# WenMind: A Comprehensive Benchmark for Evaluating Large Language Models in Chinese Classical Literature and Language Arts

**Jiahuan Cao**[†1,3]**, Yang Liu**[†1,3]**, Yongxin Shi**[1,3]**, Kai Ding**[2,3]**, Lianwen Jin**[*1,3]
[1]South China University of Technology
[2]INTSIG Information Co., Ltd
[3]INTSIG-SCUT Joint Lab on Document Analysis and Recognition
jiahuanc@foxmail.com, ly10061105@gmail.com, yongxin_shi@foxmail.com
danny_ding@intsig.net, eelwjin@scut.edu.cn

## Abstract

Large Language Models (LLMs) have made significant advancements across numerous domains, but their capabilities in Chinese Classical Literature and Language Arts (CCLLA) remain largely unexplored due to the limited scope and tasks of existing benchmarks. To fill this gap, we propose WenMind, a comprehensive benchmark dedicated for evaluating LLMs in CCLLA. WenMind covers the sub-domains of Ancient Prose, Ancient Poetry, and Ancient Literary Culture, comprising 4,875 question-answer pairs, spanning 42 fine-grained tasks, 3 question formats, and 2 evaluation scenarios: domain-oriented and capability-oriented. Based on WenMind, we conduct a thorough evaluation of 31 representative LLMs, including general-purpose models and ancient Chinese LLMs. The results reveal that even the best-performing model, ERNIE-4.0, only achieves a total score of 64.3, indicating significant room for improvement of LLMs in the CCLLA domain. We also provide insights into the strengths and weaknesses of different LLMs and highlight the importance of pre-training data in achieving better results. Overall, WenMind serves as a standardized and comprehensive baseline, providing valuable insights for future CCLLA research. Our benchmark and related code are available at `https://github.com/SCUT-DLVCLab/WenMind`.

## 1 Introduction

The emergence of Large Language Models (LLMs) has led to significant advancements in natural language processing and understanding across a wide range of domains, from finance [1, 2, 3] and law [4, 5, 6] to healthcare [7, 8, 9]. However, the capabilities of LLMs in the domain of Chinese Classical Literature and Language Arts (CCLLA) have not been fully explored, largely due to the lack of comprehensive benchmarks. CCLLA, which encompasses the sub-domains of Ancient Prose, Ancient Poetry, and Ancient Literary Culture, serves as an essential bridge connecting ancient Chinese culture to the modern world. Evaluating and improving the performance of LLMs in this domain is crucial for furthering research and applications in CCLLA. Unfortunately, existing benchmarks focus primarily on the Ancient Prose sub-domain or contain only a limited number of tasks, making it challenging to provide a thorough and holistic assessment of LLMs' capabilities in CCLLA.

To address this gap, we introduce the WenMind benchmark for a comprehensive evaluation of LLMs' CCLLA capacities. Our WenMind offers several benefits over existing CCLLA benchmarks. **(a)**

---

[†]Equal contribution
[*]Corresponding author

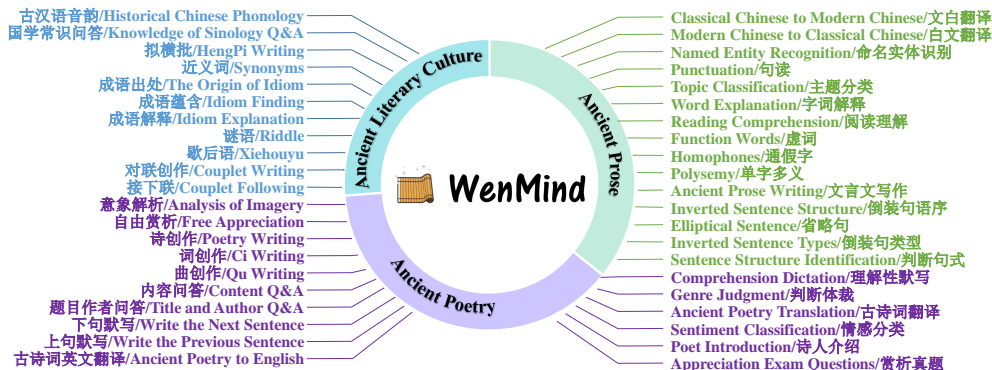

Figure 1: Overview of WenMind Benchmark, which covers 3 sub-domains and 42 fine-gained tasks.

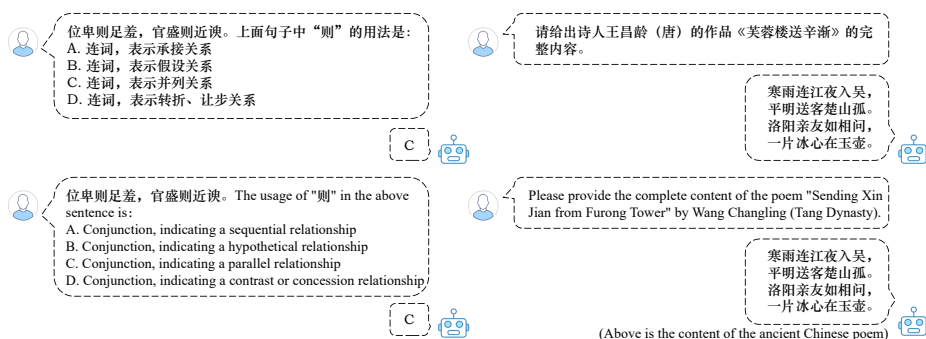

Figure 2: Samples of WenMind's data.

**Comprehensive coverage**: Unlike current benchmarks that concentrate primarily on Ancient Prose, WenMind provides a holistic approach by encompassing all three sub-domains. **(b) Substantial number of tasks**: As depicted in Figure 1, WenMind consists of 42 fine-grained tasks, which is a remarkable twofold increase over the largest existing benchmark. **(c) Variety of question formats**: WenMind incorporates a wide array of question formats, including multiple choice, fill-in-the-blank, and open-ended questions. This diversity allows for a more detailed analysis of an LLM's grasp of CCLLA. The comparison between WenMind and other benchmarks is shown in Table 1. Figure 2 presents samples of WenMind's data.

Using WenMind, we thoroughly evaluate 31 representative LLMs, including general models in Chinese and English, as well as specialized models for ancient Chinese. Our findings indicate that even the top-performing model, ERNIE-4.0 [10], only achieves the highest score of 64.3, suggesting considerable room for improvement in the CCLLA domain. Further analysis shows that the lack of knowledge of CCLLA is the main reason for existing LLMs' poor performance. Unexpectedly, LLMs specifically for ancient Chinese rather underperform the general ones. We think this is because the incremental pre-training corpus is not extensive enough to cover the scope of CCLLA, and concurrently leads to a catastrophic forgetting of the generic knowledge. Furthermore, we compare traditional metrics with model scoring metric in translation and punctuation tasks, providing a comparative analysis of their respective merits and limitations.

In summary, our contributions are as follows:

- We introduce WenMind, a novel comprehensive evaluation benchmark specific for LLMs in CCLLA, which covers all three sub-domains in this domain and contains a diverse range of tasks and question formats, facilitating a rigorous and thorough evaluation of LLMs' capabilities.

- Using WenMind, we conduct an extensive and thorough evaluation of 31 representative LLMs, revealing and quantifying their performance in the CCLLA domain.

- We perform an in-depth analysis of the evaluation results and obtain valuable insights, offering significant guidance and profound understanding for future research on LLMs in the CCLLA domain.

Table 1: Comparison of existing datasets. "CCLLA" represents "Chinese Classical Literature and Language Arts"; "QF" represents "Question Fromat"; "TM" represents "Traditional Metrics"; "MSM" represents "Model Scoring Metric"; "Method" represents "Construction Method of dataset"; "HG" represents "Human Generated"; "CI" represents "Collection and Improvement of existing datasets"; "MC" represents "Model Constructed". Datasets with the domain "General" only count relevant data entries in the CCLLA field.

| Dataset | Year | Domain | License | Scale | # Tasks | # QF | Metric | | Method | | |
|---|---|---|---|---|---|---|---|---|---|---|---|
| | | | | | | | TM | MSM | HG | CI | MC |
| C-Eval [11] | 2023 | General | CC BY-NC-SA-4.0 | 457 | 2 | 1 | ✔ | ✗ | ✔ | ✗ | ✔ |
| CIF-Bench [12] | 2024 | General | - | 150 | 3 | 1 | ✔ | ✔ | ✔ | ✔ | ✗ |
| CMMLU [13] | 2023 | General | CC BY-NC-4.0 | 620 | 3 | 1 | ✔ | ✗ | ✔ | ✗ | ✗ |
| GAOKAO-Bench [14] | 2023 | General | Apache-2.0 | 145 | 4 | 3 | ✔ | ✔ | ✔ | ✗ | ✗ |
| XiezhiBenchmark [15] | 2023 | General | CC BY-NC-SA-4.0 | 2,060 | 2 | 1 | ✔ | ✗ | ✔ | ✗ | ✔ |
| ACLUE [16] | 2023 | CCLLA | CC BY-NC-4.0 | 4,967 | 15 | 1 | ✔ | ✗ | ✔ | ✔ | ✗ |
| C-CLUE [17] | 2021 | CCLLA | CC BY-SA-4.0 | 1,122 | 3 | 1 | ✔ | ✗ | ✔ | ✗ | ✗ |
| CCLUE [18] | 2021 | CCLLA | Apache-2.0 | 36,319 | 5 | 1 | ✔ | ✗ | ✔ | ✔ | ✗ |
| CCPM [19] | 2021 | CCLLA | - | 2,720 | 1 | 1 | ✔ | ✗ | ✔ | ✗ | ✗ |
| THUAIPoet [20, 21, 22] | 2020 | CCLLA | - | 5,173 | 2 | 2 | ✔ | ✗ | ✔ | ✗ | ✗ |
| WYWEB [23] | 2023 | CCLLA | - | 69,700 | 9 | 2 | ✔ | ✗ | ✔ | ✔ | ✗ |
| **WenMind (Ours)** | 2024 | CCLLA | CC BY-NC-SA-4.0 | 4,875 | **42** | 3 | ✔ | ✔ | ✔ | ✔ | ✔ |

## 2 Related Work

### 2.1 General Chinese Benchmark for LLMs

To evaluate the performance of LLMs across diverse Chinese language tasks, several benchmarks have been proposed [24, 25, 26]. CLUE [27] is the first large-scale Chinese comprehension benchmark, including nine tasks such as sentence classification and reading comprehension. SuperCLUE [28] expands the evaluation to include user queries, open-ended dialogues, and closed-ended questions, focusing on real-world applications. CMMLU [13] aims to comprehensively assess the knowledge and reasoning capabilities of LLMs in Chinese, covering 67 subjects from basic to advanced levels. C-Eval [11] offers a thorough evaluation suite with 13,948 multiple-choice questions spanning 52 subjects, targeting foundational knowledge and reasoning skills. HalluQA [29] addresses the hallucination phenomenon in Chinese LLMs with 450 adversarial questions encompassing cultural and social aspects. AlignBench [30] evaluates the alignment of models across multiple dimensions. CBBQ [31], a bias benchmark, covers stereotypes and social biases relevant to Chinese culture with over 100 expert-constructed questions.

### 2.2 CCLLA Benchmark for LLMs

Unlike general Chinese benchmarks, CCLLA benchmarks are limited in number. C-CLUE [17] provides a dataset for evaluating named entity recognition and relation extraction. CCLUE [18] offers five evaluation tasks, including sequence labeling and sentence classification. ACLUE [16] proposes a benchmark to evaluate classical Chinese understanding capabilities of LLMs, with the form of multiple-choice questions. WYWEB [23] offers nine tasks, such as text classification, punctuation and machine translation. However, these existing CCLLA benchmarks contain only a small number of tasks or are organized using only a single multiple-choice question format. Compared to these datasets, WenMind covers a wider variety of tasks and formats, and employs evaluation methods that align more closely with human intuition, allowing it to more accurately and comprehensively reflect the capabilities of LLMs in CCLLA.

# 3 WenMind Benchmark

## 3.1 Task Definition

This study aims to construct a comprehensive benchmark for the evaluation of LLMs' capabilities in Chinese Classical Literature and Language Arts (CCLLA). In general, the CCLLA contains three sub-domains: Ancient Prose, Ancient Poetry, and Ancient Literary Culture. Therefore, we define the following tasks for these domains, respectively.

**Ancient Prose** Ancient prose refers to a form of classical Chinese literature, which characterized by simplicity, elegance and a high degree of freedom. It is commonly used in historical records, philosophical texts, agricultural books, *etc*. To evaluate the models' capabilities in Ancient Prose, we design a set of tasks that assess the understanding and generation, respectively. We focus on the models' **understanding capability at both sentence level and word level** through 11 tasks: Tasks that pertain to the sentence level include sentence structure, classical Chinese to modern Chinese, modern Chinese to classical Chinese, topic classification, and reading comprehension. On the other hand, tasks that pertain to the word level encompass named entity recognition, punctuation, word explanation, function words, homophones, and polysemy. For **generation** capability, we directly utilize ancient prose writing as the evaluation task.

**Ancient Poetry** Ancient poetry is a special form of classical Chinese literature, which follows a strict rhyme scheme and often expresses rich emotions in refined language. Considering the characteristics of ancient poetry, we design the following tasks to assess the ability in three dimensions of LLMs. (a) Appreciation, ancient poetry translation, sentiment classification, and ancient poetry to English, for **understanding** capability. (b) Ancient poetry writing for **generation** capability. (c) Basic Q&A, poet introduction, and analysis of imagery, for **knowledge** capability.

**Ancient Literary Culture** Ancient Literary Culture refers to literary forms other than Ancient Prose and Ancient Poetry, such as riddle, idiom, *etc*. We primarily assess the generation and knowledge capabilities of LLMs in this sub-domain. (a) For **generation** capability, we utilize couplet as the evaluation task. (b) For **knowledge** capability, we utilize 5 tasks, including idiom, riddle, Xiehouyu, historical Chinese phonology, and sinology.

## 3.2 Data Construction

The construction pipeline of WenMind includes data collection and data processing, as illustrated in Figure 3.

**Data Collection** Our data collection process includes three main sources. **(a) Internet:** We collect authentic examination questions from Chinese language exams and poetry competitions, which are carefully curated by experts and scholars, to serve as part of our data. Additionally, we gather texts that are enriched with CCLLA knowledge from the Internet, such as introductions of poets and interpretations of idioms. Utilize the knowledge in these texts, we manually construct question-answer pairs for various tasks. **(b) Open-Source Datasets:** Leveraging open-source corpora like C2MChn [32] and the Daizhige Corpus [33] as foundational resources, we perform a series of operations, including text filtering, question crafting, and format standardization, to generate question-answer pairs. Given the challenge of acquiring data for certain tasks and to ensure a thorough and holistic evaluation within the CCLLA domain, we also incorporate test cases from other open-source datasets, such as ACLUE [16], WYWEB [23], and THU-FSPC [22]. **(c) LLM:** Tasks such as ancient poetry and prose writing are inherently open-ended and do not have fixed correct answers. For these tasks, we initially design a range of questions and then employ the ERNIE-3.5 model [10] to generate reference answers. These answers are further reviewed, filtered, and refined through a manual process. It is notable that the WenMind benchmark is released under the CC-BY-NC-SA-4.0 license and strictly adheres to the agreements of the original data sources. For more details, please refer to Appendix B.

**Data Processing** To ensure the data quality, we perform a series of data processing on the collected data. **(a) Question Segmentation.** Some question-answer pairs consist of multiple questions intertwined. We employ the ERNIE-3.5 model [10] to distinguish between multiple questions and answers mixed together. **(b) Conversion and Standardization of Question-Answer Pairs.** We

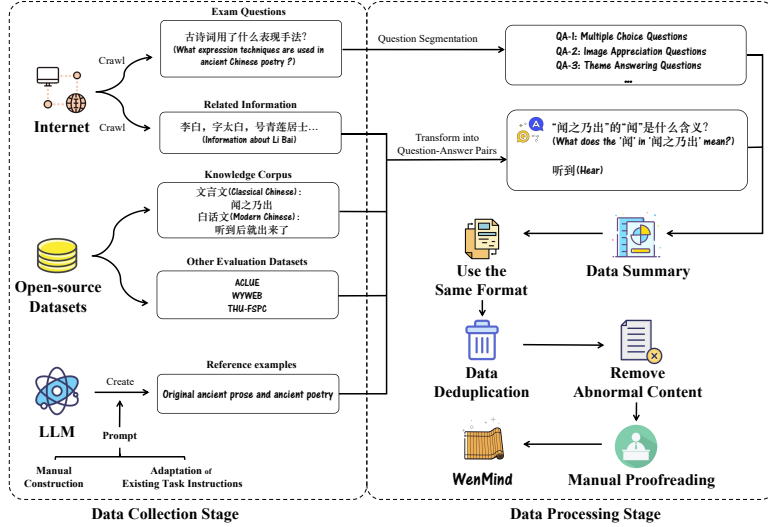

Figure 3: Construction pipeline of WenMind Benchmark. Zoom in for better view.

ensure that the data is presented in question-answer pairs, standardized in a dictionary format. Each sample is accompanied by metadata, including the task name and the capability being assessed. **(c) Data Deduplication.** A combination of MinHashLSH [34] and field-matching methods is employed to remove duplicate questions. **(d) Removal of Irrelevant Symbols and Content.** We utilize regular expressions, detection of irrelevant characters, and other methods to eliminate unnecessary English characters, abnormal symbols, erroneous data, *etc* [35, 36, 37]. **(e) Manual Proofreading.** We manually verify all question-answer pairs to ensure the integrity of the questions and the correctness of the answers.

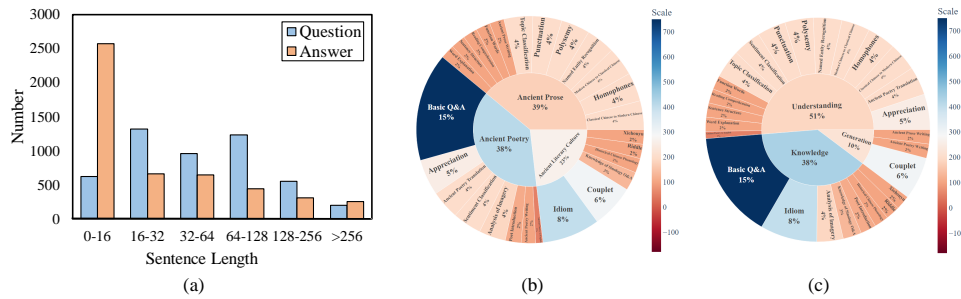

Figure 4: Data statistics of WenMind: Distributions of (a) sentence length, (b) sub-domains and (c) capabilities. Zoom in for better view.

## 3.3 Data Statistics

We present the statistics of WenMind in Figure 4 and Table 2. The WenMind benchmark comprises a total of 4,875 entries, encompassing 26 coarse-grained tasks and 42 fine-grained tasks. Tasks involving Ancient Prose, Ancient Poetry, and Ancient Literary Culture respectively account for 39%, 38%, and 23% of the benchmark, showing a relatively balanced distribution. Among these, the task with the highest number of entries is basic Q&A on ancient poetry, constituting approximately 15% of the benchmark, while tasks such as English translations of ancient poetry have the fewest entries, at around 1%. This reflects the benchmark's design, which prioritizes tasks based on their common occurrence and general applicability. From the question format perspective, WenMind comprises 3,928 open-ended questions, 917 multiple-choice questions, and 30 fill-in-the-blank

questions. Regarding cognitive capabilities, questions assessing understanding, generation, and knowledge account for 51%, 10%, and 38% respectively.

Table 2: The statistics of the WenMind Benchmrak. "Q" represents "Question" and "A" represents "Answer".

| Domain | Tasks | #Q | Max. #Q | Min. #Q | Avg. Q Tokens | Avg. A Tokens |
|---|---|---|---|---|---|---|
| Ancient Prose | 15 | 1,900 | 200 | 7 | 107.51 | 62.12 |
| Ancient Poetry | 16 | 1,845 | 200 | 20 | 73.42 | 94.93 |
| Ancient Literary Culture | 11 | 1,130 | 100 | 100 | 26.68 | 14.26 |
| **Overall** | 42 | 4,875 | 200 | 7 | 75.87 | 63.44 |

# 4 Experiment

## 4.1 Models

We conduct an extensive evaluation of 31 models, both proprietary and open-source, encompassing English-centric models such as GPT-4 [38] and LLaMA [39, 40], Chinese-centric models like Qwen [41], Baichuan [42] and ChatGLM [43, 44], models fine-tuned from English to Chinese such as LLaMA-Chinese [45, 46], and specialized models for ancient Chinese, including Xunzi [47] and Chunhua [48]. Additionally, we assess different-sized variants within the same model family to reveal the effect of model size on their performance in the CCLLA domain. Details of the evaluated models are presented in Appendix C.

## 4.2 Experiment Setup

For open-source and closed-source models we evaluate them through local access and API calls, respectively. To ensure a fair comparison, we standardize the inference settings for all evaluated LLMs. Specifically, we employ half-precision inference with bf16 and greedy decoding strategy with a maximum generation length set to 2048. The temperature parameter, Top-p sampling, and Top-k sampling are set to 1, 1, and 50, respectively. To evaluate the knowledge of LLM itself, we prohibit the use of external search engines for closed-source models. For model scoring, we randomly sample 100 instances and score them using LLMs three times, achieving an average error within 2%. This level of precision indicates that LLMs exhibit scoring stability, and therefore, we decide not to pursue multiple averaging scores due to cost considerations. All experiments are conducted on a NVIDIA A6000 GPU.

## 4.3 Evaluation Metrics

**Model Scoring Metric**    The evaluation of the WenMind benchmark mainly employs model scoring. An optimal scoring model should meet the following criteria: (a) Exhibit excellent instruction-following capabilities, generating scores based on specified prompts and outputting them in the required format. (b) Possess extensive knowledge in the CCLLA domain to aid the scoring process. (c) Balance scoring effectiveness and cost efficiency. (d) Align closely with human subjective judgment in scoring results. After experimentation, the ERNIE-3.5 model [10] has been selected as the experimental scoring model, as it exhibits a consistency of approximately 89.4% with human evaluation, meeting the required criteria. More details can be found in Appendix C.

Furthermore, the scoring requirements vary for different formats of questions. (a) For multiple-choice questions with only one correct option, the model gives a score of 0 (incorrect) or 1 (correct). (b) For multiple-choice questions with multiple correct options, the model gives a score of 0 (incorrect options present), 0.5 (partially correct options), or 1 (all options correct). (c) For subjective questions without a standard answer, the model scores between 0 and 1 based on the given requirements. (d) For Q&A questions with a standard answer, the model identifies several scoring points ($P_{all}$) based on the reference answer and then determines the number of these points present in the LLM's response ($P_{obtain}$) to calculate the score for the question($P_{obtain}/P_{all}$). All indicators are multiplied by one hundred to obtain final scores. The scoring prompts for various tasks refer to Appendix C.2.

**Traditional Metrics**    In tasks with strong objectivity or where scoring models struggle to extract the correct answers, we provide traditional metric as a reference. For punctuation task, we use the F1-score [49]. For the translation tasks of classical Chinese to modern Chinese, modern Chinese to classical Chinese, classical poetry to modern Chinese, and classical poetry to English, we utilize BLEU [50].

## 4.4   Results and Analysis

This experiment evaluates the performance of 31 models on 26 coarse-grained tasks, as shown in Table 3 and 4. Detailed metrics for LLMs on 42 fine-grained tasks are provided in Appendix C. The tasks represented by T1-T26 are as follows: T1-Sentence structure, T2-Classical Chinese to modern Chinese, T3-Modern Chinese to classical Chinese, T4-Named entity recognition, T5-Punctuation, T6-Topic classification, T7-Word explanation, T8-Reading comprehension, T9-Function words, T10-Homophones, T11-Polysemy, T12-Ancient prose writing, T13-Appreciation, T14-Ancient poetry writing, T15-Basic Q&A, T16-Ancient poetry translation, T17-Sentiment classification, T18-Ancient poetry to English, T19-Poet introduction, T20-Analysis of imagery, T21-Couplet, T22-Idiom, T23-Riddle, T24-Xiehouyu, T25-Historical Chinese phonology, T26-Knowledge of sinology Q&A. Based on the results, we draw the following insights.

**There is considerable room for improvement of existing LLMs' CCLLA capabilities.** ERNIE-4.0 [10] performs the best with a score of 64.3. LLaMA2-7B-Chat [40] has the lowest score, only 13.0. The average score of the 31 LLMs is approximately 41.2. Most models have scores between 20 and 60, with over 64% of the LLMs scoring below 50. Overall, the scores are relatively low, suggesting there is considerable room for improvement.

**Pre-training data matters.** General-purpose Chinese models demonstrate superior performance in the CCLLA domain. Additionally, fine-tuning general-purpose English models with Chinese data improves their performance, yet such enhancements do not fully match the level of proficiency of the general-purpose Chinese model, highlighting the significance of Chinese pre-training data in the CCLLA domain.

**Incremental pre-training in the CCLLA domain may not be effective.** Unexpectedly, the models specifically designed for ancient Chinese with incremental pre-training and fine-tuning, namely Ancient-Chat-LLM-7B [51], Bloom-7B-Chunhua [48], and Xunzi-Qwen1.5-7B [47], show subpar performance in the CCLLA domain, achieving an average score of 34.1. This indicates that incremental pre-training in the CCLLA domain may not be effective. The possible reason could be the incremental pre-training and fine-tuning data being insufficient to cover a wide range of knowledge and tasks, which concurrently leads to more profound catastrophic forgetting of CCLLA-related knowledge from the pre-training corpus.

**Large Language Models lack sufficient knowledge in the CCLLA domain.**  As shown in Table 4, LLMs exhibit substantially different performance across various domains and capabilities dimension. On different capability dimensions, LLMs perform significantly worse in knowledge than in generation and understanding, indicating that LLMs lack sufficient knowledge in the CCLLA domain. Moreover, the fields of Ancient Poetry and Ancient Literary Culture contain more tasks with a knowledge dimension, which leads to LLMs performing noticeably worse in these two domains compared to Ancient Prose.

**The principle of scaling law remains valid in the CCLLA domain.** We present the relationship between model performance and parameters of Qwen [41] and Yi [52] in Figure 5, which shows that the performance improves with the parameters increasing, illustrating the scaling law [53] for LLMs in CCLLA domain.

For the tasks of translation and punctuation, we further utilize BLEU [50] and F1-score [49] as a traditional metric, respectively, assessing the strengths and weaknesses of traditional metrics versus model scoring metric. Specifically, in punctuation and four translation tasks, we select five models and for each model, we randomly choose 300 samples for manual evaluation to determine which metric aligns more closely with human evaluation. The comparative results between traditional metrics and model scoring metric are illustrated in Figure 6. It can be observed that for tasks with singular and definitive answers such as punctuation, traditional metrics are more appropriate as the scoring model struggle to extract the correct answer from responses of LLMs. In contrast, for tasks with non-unique answers like translation, model scoring metric demonstrates a high degree

Figure 5: The relationship between model performance and parameters.

Table 3: Results of all evaluated models on WenMind. The tasks represented by T1-T26 are detailed in Section 4.4, and details of the evaluated models are presented in Appendix C. Zoom in for better view.

| Model | Overall | Ancient Prose | | | | | | | | | | | | Ancient Poetry | | | | | | | | Ancient Literary Culture | | | | | |
|---|---|---|---|---|---|---|---|---|---|---|---|---|---|---|---|---|---|---|---|---|---|---|---|---|---|---|---|
| | | T1 | T2 | T3 | T4 | T5 | T6 | T7 | T8 | T9 | T10 | T11 | T12 | T13 | T14 | T15 | T16 | T17 | T18 | T19 | T20 | T21 | T22 | T23 | T24 | T25 | T26 |
| Baichuan2-7B-Chat [42] | 41.2 | 33.2 | 56.8 | 58.6 | 68.1 | 73.1 | 30.5 | 46.3 | 67.1 | 44.0 | 16.0 | 47.0 | 50.6 | 54.5 | 59.0 | 17.5 | 62.1 | 10.5 | 47.1 | 46.5 | 39.2 | 60.5 | 29.0 | 16.0 | 20.0 | 35.0 | 59.5 |
| Baichuan2-13B-Chat [42] | 45.5 | 48.5 | 53.8 | 62.5 | 64.3 | 72.5 | 37.0 | 56.5 | 77.6 | 41.0 | 24.0 | 51.0 | 60.7 | 57.2 | 63.5 | 20.1 | 66.9 | 43.0 | 51.3 | 55.4 | 38.9 | 56.0 | 29.8 | 26.0 | 24.0 | 33.0 | 76.7 |
| Firefly-Baichuan2-13B [54] | 38.7 | 14.6 | 46.8 | 29.5 | 62.5 | 75.1 | 38.5 | 41.8 | 63.8 | 30.0 | 21.5 | 53.5 | 43.7 | 43.4 | 40.4 | 19.8 | 61.8 | 35.0 | 48.8 | 39.0 | 28.7 | 55.7 | 26.2 | 24.0 | 14.5 | 32.0 | 65.2 |
| ChatGLM2-6B [43] | 35.4 | 36.0 | 46.2 | 62.7 | 60.8 | 62.5 | 31.0 | 36.7 | 55.1 | 33.0 | 5.0 | 43.0 | 51.5 | 42.0 | 55.2 | 12.3 | 48.8 | 45.0 | 39.1 | 30.6 | 31.4 | 51.6 | 23.4 | 6.0 | 9.5 | 16.0 | 45.2 |
| ChatGLM3-6B [43] | 39.5 | 44.0 | 52.2 | 60.7 | 68.1 | 74.9 | 41.0 | 46.3 | 64.2 | 42.0 | 22.0 | 40.0 | 53.4 | 55.3 | 54.5 | 10.6 | 55.5 | 43.0 | 44.9 | 33.3 | 36.3 | 56.2 | 23.7 | 2.0 | 4.0 | 36.0 | 42.8 |
| InternLM2-Chat-7B [55] | 50.2 | 43.5 | 61.1 | 54.4 | 42.6 | 77.6 | 42.5 | 65.8 | 79.6 | 48.0 | 23.0 | 60.0 | 55.5 | 68.1 | 54.6 | 28.9 | 66.2 | 40.0 | 51.4 | 62.4 | 69.8 | 68.7 | 47.8 | 7.0 | 21.0 | 28.0 | 79.9 |
| Qwen1.5-0.5B-Chat [41] | 26.1 | 42.3 | 50.8 | 64.3 | 16.1 | 58.7 | 32.5 | 15.5 | 31.5 | 31.0 | 12.0 | 39.0 | 29.4 | 27.5 | 16.3 | 3.1 | 62.9 | 30.0 | 19.9 | 5.3 | 1.8 | 57.1 | 8.1 | 3.0 | 2.0 | 24.0 | 24.4 |
| Qwen1.5-4B-Chat [41] | 39.6 | 36.8 | 57.1 | 56.7 | 60.1 | 71.8 | 32.5 | 42.3 | 49.8 | 47.0 | 20.0 | 52.0 | 44.5 | 46.4 | 38.4 | 21.5 | 57.8 | 34.5 | 43.9 | 27.3 | 25.6 | 59.9 | 22.3 | 7.5 | 12.0 | 30.0 | 69.1 |
| Qwen1.5-7B-Chat [41] | 50.3 | 46.3 | 57.5 | 55.8 | 60.4 | 73.0 | 37.8 | 68.1 | 82.9 | 62.0 | 25.5 | 53.5 | 68.2 | 68.5 | 65.6 | 24.0 | 65.0 | 64.5 | 59.4 | 58.7 | 64.6 | 63.7 | 36.9 | 18.5 | 21.0 | 30.0 | 74.7 |
| Qwen1.5-14B-Chat [41] | 54.9 | 36.0 | 62.3 | 54.5 | 70.2 | 80.3 | 48.0 | 69.7 | 84.4 | 66.0 | 30.5 | 67.5 | 66.1 | 72.9 | 65.6 | 30.8 | 68.2 | 67.5 | 52.4 | 63.3 | 68.7 | 65.0 | 40.8 | 25.0 | 25.5 | 39.0 | 82.1 |
| Qwen1.5-32B-Chat [41] | 57.0 | 35.7 | 61.8 | 54.9 | 75.4 | 79.4 | 51.0 | 69.3 | 85.2 | 68.0 | 46.5 | 69.5 | 67.5 | 75.7 | 64.7 | 32.0 | 67.9 | 64.0 | 54.7 | 58.1 | 65.6 | 65.5 | 48.0 | 33.0 | 30.5 | 43.0 | 83.2 |
| Qwen1.5-72B-Chat [41] | 58.5 | 46.5 | 64.6 | 58.6 | 72.3 | 82.0 | 53.0 | 68.7 | 83.2 | 69.0 | 41.0 | 69.0 | 68.9 | 77.5 | 67.3 | 32.8 | 69.0 | 69.5 | 58.5 | 66.1 | 75.8 | 66.9 | 46.6 | 41.0 | 27.0 | 45.0 | 84.3 |
| Yi-1.5-6B-Chat [52] | 47.2 | 37.0 | 57.7 | 65.9 | 64.4 | 69.9 | 36.3 | 60.3 | 80.7 | 47.0 | 17.5 | 51.5 | 64.0 | 64.8 | 61.3 | 19.9 | 62.1 | 53.0 | 48.4 | 56.7 | 55.9 | 61.5 | 40.6 | 7.0 | 17.0 | 23.0 | 77.2 |
| Yi-1.5-9B-Chat [52] | 51.7 | 36.9 | 62.5 | 58.0 | 65.7 | 75.7 | 48.3 | 72.0 | 89.7 | 62.0 | 24.0 | 57.5 | 65.6 | 66.4 | 63.0 | 25.3 | 65.9 | 55.0 | 53.0 | 58.1 | 58.3 | 65.6 | 48.7 | 10.0 | 16.0 | 34.0 | 75.0 |
| Yi-1.5-34B-Chat [52] | 57.4 | 55.0 | 61.4 | 64.0 | 70.1 | 75.0 | 56.0 | 70.0 | 86.8 | 77.0 | 31.5 | 62.0 | 69.0 | 71.5 | 66.6 | 30.5 | 69.0 | 53.5 | 54.2 | 64.6 | 77.2 | 70.8 | 56.0 | 19.5 | 24.0 | 50.0 | 83.9 |
| ERNIE-3.5-8K-0329 [10] | 62.2 | 56.0 | 59.0 | 35.7 | 81.3 | 86.2 | 48.5 | 78.7 | 90.0 | 70.0 | 38.5 | 72.0 | 69.1 | 75.8 | 65.0 | 36.7 | 77.5 | 50.5 | 56.9 | 71.0 | 73.6 | 80.1 | 61.5 | 66.3 | 78.5 | 54.0 | 87.4 |
| ERNIE-4.0-8K-0329 [10] | 64.3 | 50.0 | 62.8 | 48.0 | 76.2 | 85.2 | 53.1 | 84.0 | 88.9 | 73.0 | 46.5 | 75.0 | 70.8 | 79.4 | 65.7 | 36.3 | 65.0 | 61.5 | 55.9 | 69.5 | 81.3 | 81.4 | 62.1 | 64.0 | 84.5 | 67.0 | 93.4 |
| Spark-3.5 [56] | 60.9 | 33.0 | 60.8 | 36.0 | 77.1 | 85.1 | 49.8 | 81.3 | 79.9 | 61.0 | 33.0 | 63.0 | 71.2 | 67.1 | 61.2 | 37.7 | 61.4 | 59.0 | 60.9 | 72.2 | 73.3 | 67.3 | 80.2 | 67.0 | 70.0 | 54.0 | 91.5 |
| Gemma-1.1-7B-IT [57] | 25.2 | 10.5 | 24.2 | 52.0 | 61.9 | 23.9 | 29.2 | 18.0 | 38.3 | 19.0 | 15.5 | 35.5 | 44.8 | 29.7 | 45.5 | 1.2 | 45.8 | 58.0 | 41.1 | 7.7 | 20.0 | 49.4 | 7.5 | 2.0 | 0.0 | 23.0 | 5.1 |
| Ziya-LLaMA-13B-v1.1 [58] | 34.1 | 36.5 | 59.0 | 60.5 | 8.3 | 73.3 | 23.5 | 42.3 | 57.3 | 36.0 | 28.0 | 40.0 | 50.3 | 46.0 | 50.0 | 6.4 | 57.5 | 40.5 | 40.2 | 31.6 | 30.7 | 50.2 | 23.1 | 3.0 | 6.0 | 24.0 | 44.1 |
| LLaMA2-7B-Chat [40] | 13.0 | 0.0 | 12.0 | 23.2 | 33.2 | 0.5 | 13.5 | 8.5 | 31.5 | 12.0 | 1.0 | 3.5 | 39.5 | 25.5 | 37.0 | 0.3 | 29.3 | 29.5 | 25.3 | 4.0 | 14.3 | 19.3 | 8.8 | 0.0 | 0.0 | 3.0 | 6.2 |
| LLaMA2-13B-Chat [40] | 23.7 | 6.8 | 31.1 | 43.5 | 46.1 | 31.9 | 19.5 | 22.3 | 46.3 | 19.0 | 7.0 | 28.0 | 54.7 | 29.8 | 46.1 | 0.7 | 47.4 | 53.0 | 43.1 | 9.4 | 21.4 | 33.9 | 12.9 | 2.0 | 0.0 | 20.0 | 14.3 |
| LLaMA2-Chinese-7B-Chat [45] | 18.1 | 3.5 | 22.2 | 49.4 | 63.9 | 41.1 | 22.5 | 11.3 | 29.5 | 22.0 | 3.0 | 28.5 | 34.8 | 20.5 | 18.8 | 0.1 | 27.0 | 20.5 | 29.0 | 5.4 | 10.8 | 24.0 | 5.9 | 2.0 | 0.0 | 11.0 | 3.1 |
| LLaMA2-Chinese-13B-Chat [46] | 23.7 | 4.8 | 32.5 | 53.6 | 66.3 | 68.0 | 20.5 | 13.5 | 42.5 | 28.0 | 12.5 | 29.5 | 36.1 | 27.4 | 22.2 | 0.7 | 36.3 | 39.5 | 37.7 | 1.2 | 8.3 | 39.5 | 6.6 | 1.0 | 1.0 | 19.0 | 12.0 |
| LLaMA3-8B-Instruct [59] | 34.7 | 33.3 | 52.4 | 57.6 | 53.4 | 44.4 | 42.9 | 42.0 | 62.7 | 39.0 | 16.0 | 44.0 | 57.5 | 47.0 | 51.6 | 1.8 | 53.8 | 50.0 | 59.3 | 23.4 | 33.7 | 59.3 | 22.3 | 4.0 | 2.0 | 22.0 | 25.7 |
| LLaMA3-Chinese-8B-Chat [60] | 37.3 | 34.0 | 46.3 | 61.2 | 60.2 | 68.5 | 45.5 | 50.8 | 70.4 | 36.0 | 23.0 | 45.0 | 58.4 | 50.9 | 51.3 | 1.7 | 62.4 | 42.5 | 52.3 | 23.4 | 55.7 | 56.3 | 21.8 | 3.0 | 4.0 | 22.0 | 21.5 |
| GPT-3.5 [61] | 35.3 | 41.0 | 46.8 | 54.4 | 40.4 | 72.0 | 42.0 | 39.7 | 63.2 | 21.0 | 24.5 | 47.5 | 56.2 | 46.8 | 55.0 | 6.0 | 59.4 | 48.5 | 49.7 | 21.9 | 43.8 | 47.5 | 16.6 | 4.0 | 4.0 | 24.0 | 32.9 |
| GPT-4 [62] | 50.2 | 26.5 | 63.5 | 57.0 | 65.0 | 75.0 | 57.0 | 66.7 | 86.1 | 52.0 | 38.5 | 67.5 | 67.9 | 64.3 | 64.9 | 16.7 | 70.2 | 61.5 | 60.1 | 48.2 | 63.9 | 58.6 | 38.7 | 16.5 | 13.0 | 37.0 | 69.1 |
| Ancient-Chat-LLM-7B [51] | 32.7 | 40.5 | 54.8 | 45.1 | 28.6 | 76.5 | 37.5 | 24.5 | 42.6 | 41.0 | 26.5 | 50.0 | 23.5 | 19.3 | 31.1 | 14.7 | 52.7 | 36.0 | 28.6 | 23.9 | 17.9 | 46.9 | 17.6 | 8.0 | 8.0 | 39.0 | 59.9 |
| Bloom-7B-Chunhua [48] | 32.5 | 17.5 | 50.1 | 56.1 | 71.7 | 77.7 | 23.0 | 32.8 | 43.8 | 29.0 | 12.5 | 40.0 | 25.3 | 28.3 | 39.2 | 9.3 | 56.5 | 32.5 | 39.6 | 27.1 | 19.3 | 47.5 | 22.2 | 5.0 | 8.5 | 30.0 | 43.4 |
| Xunzi-Qwen1.5-7B [47] | 37.0 | 37.8 | 63.4 | 51.0 | 23.5 | 84.2 | 29.5 | 35.0 | 43.1 | 54.0 | 21.0 | 48.0 | 40.2 | 30.7 | 40.8 | 18.3 | 60.6 | 45.5 | 43.9 | 33.6 | 8.6 | 51.1 | 23.9 | 19.0 | 19.5 | 35.0 | 67.0 |
| Average | 41.2 | 33.0 | 51.4 | 53.1 | 57.4 | 67.6 | 37.9 | 47.8 | 64.4 | 44.5 | 22.8 | 49.5 | 53.5 | 51.0 | 51.1 | 16.7 | 58.4 | 46.3 | 46.8 | 38.7 | 42.4 | 56.0 | 31.0 | 16.5 | 18.3 | 31.7 | 54.8 |

of consistency with human assessment. It is noteworthy that most models are unable to translate modern Chinese into classical Chinese, merely rephrasing the modern text. Consequently, the scoring model, which operate from a semantic perspective, incorrectly assign high scores, making the use of traditional metrics more rational for this task. More details can be found in Appendix C.

Figure 6: Comparative results between traditional metrics and model scoring metric.

## 5 Conclusion

In this paper, we propose WenMind, a comprehensive benchmark for the evaluation of Large Language Models (LLMs) in the domain of Chinese Classical Literature and Language Arts (CCLLA). WenMind covers all three sub-domains in CCLLA and contains 42 tasks and 3 question formats, exhibits advantages over existing benchmarks. Leveraging WenMind, we comprehensively evaluate

Table 4: Results of all evaluated models on different domains and capabilities. Details of the evaluated models are presented in Appendix C.

| Model | Overall | Domain | | | Capability | | |
|---|---|---|---|---|---|---|---|
| | | Ancient Prose | Ancient Poetry | Ancient Literary Culture | Understanding | Generation | Knowledge |
| Baichuan2-7B-Chat [42] | 41.2 | 49.5 | 33.6 | 39.5 | 47.8 | 58.2 | 27.7 |
| Baichuan2-13B-Chat [42] | 45.5 | 53.4 | 39.8 | 41.6 | 53.7 | 58.4 | 31.2 |
| Firefly-Baichuan2-13B [54] | 38.7 | 44.7 | 33.1 | 37.8 | 45.2 | 50.2 | 26.9 |
| ChatGLM2-6B [43] | 35.4 | 43.9 | 29.9 | 30.0 | 43.8 | 52.3 | 19.6 |
| ChatGLM3-6B [43] | 39.5 | 50.9 | 32.4 | 32.0 | 50.9 | 55.7 | 20.0 |
| InternLM2-Chat-7B [55] | 50.2 | 53.4 | 47.5 | 49.3 | 54.7 | 63.3 | 40.8 |
| Qwen1.5-0.5B-Chat [41] | 26.1 | 36.7 | 17.0 | 23.4 | 37.2 | 43.4 | 6.7 |
| Qwen1.5-4B-Chat [41] | 39.6 | 48.5 | 32.5 | 36.1 | 48.0 | 52.5 | 24.9 |
| Qwen1.5-7B-Chat [41] | 50.3 | 55.5 | 48.2 | 44.7 | 57.9 | 65.0 | 36.2 |
| Qwen1.5-14B-Chat [41] | 54.9 | 60.5 | 52.8 | 49.1 | 62.5 | 65.3 | 42.0 |
| Qwen1.5-32B-Chat [41] | 57.0 | 63.3 | 52.6 | 53.4 | 64.6 | 65.7 | 44.4 |
| Qwen1.5-72B-Chat [41] | 58.5 | 64.0 | 55.6 | 54.0 | 65.9 | 67.4 | 46.3 |
| Yi-1.5-6B-Chat [52] | 47.2 | 53.4 | 42.9 | 43.7 | 54.7 | 61.9 | 33.3 |
| Yi-1.5-9B-Chat [52] | 51.7 | 58.4 | 46.6 | 48.6 | 59.1 | 65.0 | 38.1 |
| Yi-1.5-34B-Chat [52] | 57.4 | 63.0 | 52.0 | 56.6 | 63.2 | 69.6 | 46.4 |
| ERNIE-3.5-8K-0329 [10] | 62.2 | 63.5 | 55.7 | 70.7 | 64.4 | 74.8 | 55.9 |
| ERNIE-4.0-8K-0329 [10] | 64.3 | 66.3 | 56.6 | 73.4 | 66.8 | 76.1 | 57.8 |
| Spark-3.5 [56] | 60.9 | 59.8 | 54.1 | 73.7 | 60.2 | 66.9 | 60.2 |
| Gemma-1.1-7B-IT [57] | 25.2 | 32.4 | 21.8 | 18.6 | 34.9 | 47.7 | 6.2 |
| Ziya-LLaMA-13B-v1.1 [58] | 34.1 | 42.5 | 28.2 | 29.5 | 43.5 | 50.2 | 17.2 |
| LLaMA2-7B-Chat [40] | 13.0 | 14.0 | 14.3 | 9.2 | 16.8 | 26.9 | 4.2 |
| LLaMA2-13B-Chat [40] | 23.7 | 29.7 | 21.6 | 17.1 | 32.2 | 40.5 | 7.9 |
| LLaMA2-Chinese-7B-Chat [45] | 18.1 | 29.6 | 11.2 | 10.0 | 27.5 | 25.1 | 3.6 |
| LLaMA2-Chinese-13B-Chat [46] | 23.7 | 36.4 | 15.3 | 16.0 | 35.7 | 35.3 | 4.5 |
| LLaMA3-8B-Instruct [59] | 34.7 | 45.0 | 27.5 | 29.1 | 46.1 | 57.4 | 13.4 |
| LLaMA3-Chinese-8B-Chat [60] | 37.3 | 49.9 | 30.1 | 27.7 | 50.2 | 55.7 | 15.2 |
| GPT-3.5 [61] | 35.3 | 46.1 | 30.5 | 25.1 | 47.1 | 50.7 | 15.6 |
| GPT-4 [62] | 50.2 | 60.3 | 44.2 | 43.1 | 61.3 | 61.7 | 32.4 |
| Ancient-Chat-LLM-7B [51] | 32.7 | 42.6 | 23.9 | 30.5 | 41.1 | 39.1 | 19.9 |
| Bloom-7B-Chunhua [48] | 32.5 | 42.7 | 24.0 | 29.3 | 42.2 | 41.4 | 17.3 |
| Xunzi-Qwen1.5-7B [47] | 37.0 | 44.8 | 29.4 | 36.2 | 44.9 | 46.8 | 23.8 |
| Average | 41.2 | 48.5 | 35.6 | 38.0 | 49.2 | 54.5 | 27.1 |

Table 5: Model scoring metric and traditional metrics on translation and punctuation tasks. "TM" represents "Traditional Metrics"; "MSM" represents "Model Scoring Metric". Details of the evaluated models are presented in Appendix C. "TM" is either BLEU or F1.

| Model | T2 | | T3 | | T16 | | T18 | | T5 | |
|---|---|---|---|---|---|---|---|---|---|---|
| | MSM | TM | MSM | TM | MSM | TM | MSM | TM | MSM | TM |
| Baichuan2-13B-Chat [42] | 53.8 | 13.3 | 62.5 | 7.8 | 66.9 | 8.8 | 51.3 | 38.2 | 72.5 | 59.8 |
| ERNIE-4.0-8K-0329 [10] | 62.8 | 19.8 | 48.0 | 28.0 | 65.0 | 5.6 | 55.9 | 34.1 | 85.2 | 61.2 |
| LLaMA3-Chinese-8B-Chat [60] | 46.3 | 12.2 | 61.2 | 8.4 | 62.4 | 7.3 | 52.3 | 35.2 | 68.5 | 57.7 |
| GPT-3.5 [61] | 46.8 | 12.3 | 54.4 | 7.2 | 59.4 | 8.5 | 49.7 | 38.2 | 72.0 | 58.0 |
| Xunzi-Qwen1.5-7B [47] | 63.4 | 25.4 | 51.0 | 29.1 | 60.6 | 10.7 | 43.9 | 27.0 | 84.2 | 75.6 |

31 representative LLMs. The results reveal insights into their performance levels, highlighting areas for improvement. WenMind provides a standardized and detailed assessment, enabling researchers to assess and compare the performance of LLMs in the CCLLA domain effectively. Our study highlights the importance of knowledge enrichment in LLMs for the CCLLA domain and provides valuable insights for future research and development.

# 6 Limitations

The main limitation of our work is the use of an aligned LLMs for scoring, which might introduce some degree of error. However, we take several measures to minimize this issue. We standardize the behavior of the scoring LLMs to ensure fairness and consistency, resulting in a high level of consistency with human evaluation (89.4%). This minimizes the impact of error and ensures a reliable comparison between LLMs and human scoring. Additionally, while our evaluation tasks are designed to align closely with real-world applications of LLMs in the field of CCLLA, some tasks may not fully capture the complexity of specific application needs, such as T5 punctuation and T6 theme classification. These tasks serve more as general evaluations rather than targeted assessments of nuanced requirements in practical scenarios.

# 7 Ethical Statement

Our evaluation dataset contains content of significant historical and academic value, designed to promote scholarly research and educational applications related to historical texts, language transfor-

mation, and ancient scenarios. It is not intended to reinforce or propagate societal biases. Throughout the data collection and processing phases, we have taken measures to minimize the presence of unsafe content. For tasks involving content generation, we emphasize that the generated materials should be limited to academic research and educational use, and must not be employed for commercial purposes, political propaganda, or any objectives that may lead to bias or discrimination. In light of potential misuse risks, we explicitly restrict the use of the dataset and generated texts to prevent inappropriate applications. We acknowledge the social, cultural, and historical risks that may be inherent in the dataset, and we are committed to actively exploring and implementing strategies to mitigate such risks. With this statement, we aim to provide academic value while addressing potential ethical concerns, underscoring that the dataset must not be misused.

## Acknowledgement

This research is supported in part by National Natural Science Foundation of China (Grant No.: 62441604, 62476093).

## References

[1] Qianqian Xie, Weiguang Han, Zhengyu Chen, Ruoyu Xiang, Xiao Zhang, Yueru He, Mengxi Xiao, Dong Li, Yongfu Dai, Duanyu Feng, et al. The FinBen: An holistic financial benchmark for large language models. *arXiv preprint arXiv:2402.12659*, 2024.

[2] Liwen Zhang, Weige Cai, Zhaowei Liu, Zhi Yang, Wei Dai, Yujie Liao, Qianru Qin, Yifei Li, Xingyu Liu, Zhiqiang Liu, et al. FinEval: A Chinese financial domain knowledge evaluation benchmark for large language models. *arXiv preprint arXiv:2308.09975*, 2023.

[3] Dakuan Lu, Hengkui Wu, Jiaqing Liang, Yipei Xu, Qianyu He, Yipeng Geng, Mengkun Han, Yingsi Xin, and Yanghua Xiao. BBT-Fin: Comprehensive construction of Chinese financial domain pre-trained language model, corpus and benchmark. *arXiv preprint arXiv:2302.09432*, 2023.

[4] Yongfu Dai, Duanyu Feng, Jimin Huang, Haochen Jia, Qianqian Xie, Yifang Zhang, Weiguang Han, Wei Tian, and Hao Wang. LAiW: A Chinese legal large language models benchmark. *arXiv preprint arXiv:2310.05620*, 2023.

[5] Zhiwei Fei, Xiaoyu Shen, Dawei Zhu, Fengzhe Zhou, Zhuo Han, Songyang Zhang, Kai Chen, Zongwen Shen, and Jidong Ge. LawBench: Benchmarking legal knowledge of large language models. *arXiv preprint arXiv:2309.16289*, 2023.

[6] Neel Guha, Julian Nyarko, Daniel Ho, Christopher Ré, Adam Chilton, Alex Chohlas-Wood, Austin Peters, Brandon Waldon, Daniel Rockmore, Diego Zambrano, et al. LegalBench: A collaboratively built benchmark for measuring legal reasoning in large language models. *Advances in Neural Information Processing Systems*, 36, 2023.

[7] Amin Dada, Marie Bauer, Amanda Butler Contreras, Osman Alperen Koraş, Constantin Marc Seibold, Kaleb E Smith, and Jens Kleesiek. CLUE: A clinical language understanding evaluation for LLMs. *arXiv preprint arXiv:2404.04067*, 2024.

[8] Xidong Wang, Guiming Hardy Chen, Dingjie Song, Zhiyi Zhang, Zhihong Chen, Qingying Xiao, Feng Jiang, Jianquan Li, Xiang Wan, Benyou Wang, et al. CMB: A comprehensive medical benchmark in Chinese. *arXiv preprint arXiv:2308.08833*, 2023.

[9] Jianquan Li, Xidong Wang, Xiangbo Wu, Zhiyi Zhang, Xiaolong Xu, Jie Fu, Prayag Tiwari, Xiang Wan, and Benyou Wang. Huatuo-26M, a large-scale Chinese medical QA dataset. *arXiv preprint arXiv:2305.01526*, 2023.

[10] Baidu. ERNIE. https://yiyan.baidu.com/.

[11] Yuzhen Huang, Yuzhuo Bai, Zhihao Zhu, Junlei Zhang, Jinghan Zhang, Tangjun Su, Junteng Liu, Chuancheng Lv, Yikai Zhang, Jiayi Lei, Yao Fu, Maosong Sun, and Junxian He. C-Eval: A multi-level multi-discipline Chinese evaluation suite for foundation models. *Advances in Neural Information Processing Systems*, 2023.

[12] Yizhi Li, Ge Zhang, Xingwei Qu, Jiali Li, Zhaoqun Li, Zekun Wang, Hao Li, Ruibin Yuan, Yinghao Ma, Kai Zhang, et al. CIF-Bench: A Chinese instruction-following benchmark for evaluating the generalizability of large language models. *arXiv preprint arXiv:2402.13109*, 2024.

[13] Haonan Li, Yixuan Zhang, Fajri Koto, Yifei Yang, Hai Zhao, Yeyun Gong, Nan Duan, and Timothy Baldwin. CMMLU: Measuring massive multitask language understanding in Chinese. *arXiv preprint arXiv:2306.09212*, 2023.

[14] Xiaotian Zhang, Chunyang Li, Yi Zong, Zhengyu Ying, Liang He, and Xipeng Qiu. Evaluating the performance of large language models on GAOKAO benchmark. *arXiv preprint arXiv:2305.12474*, 2023.

[15] Zhouhong Gu, Xiaoxuan Zhu, Haoning Ye, Lin Zhang, Jianchen Wang, Yixin Zhu, Sihang Jiang, Zhuozhi Xiong, Zihan Li, Weijie Wu, et al. Xiezhi: An ever-updating benchmark for holistic domain knowledge evaluation. In *Proceedings of the AAAI Conference on Artificial Intelligence*, pages 18099–18107, 2024.

[16] Yixuan Zhang and Haonan Li. Can large langauge model comprehend ancient Chinese? A preliminary test on ACLUE. In *Proceedings of the Ancient Language Processing Workshop*, pages 80–87, Varna, Bulgaria, September 2023. INCOMA Ltd., Shoumen, Bulgaria.

[17] Zijing Ji, Yuxin Shen, Yining Sun, Tian Yu, and Xin Wang. C-CLUE: A benchmark of classical Chinese based on a crowdsourcing system for knowledge graph construction. In *Knowledge Graph and Semantic Computing: Knowledge Graph Empowers New Infrastructure Construction: 6th China Conference, CCKS 2021, Guangzhou, China, November 4-7, 2021, Proceedings 6*, pages 295–301. Springer, 2021.

[18] Ethan. CCLUE: Classical Chinese language understanding evaluation benchmark: datasets, baselines, pre-trained models, corpus and leaderboard. https://github.com/Ethan-yt/CCLUE.

[19] Wenhao Li, Fanchao Qi, Maosong Sun, Xiaoyuan Yi, and Jiarui Zhang. CCPM: A Chinese classical poetry matching dataset. *arXiv preprint arXiv:2106.01979*, 2021.

[20] Zhipeng Guo, Xiaoyuan Yi, Maosong Sun, Wenhao Li, Cheng Yang, Jiannan Liang, Huimin Chen, Yuhui Zhang, and Ruoyu Li. Jiuge: A human-machine collaborative Chinese classical poetry generation system. In *Proceedings of the 57th Annual Meeting of the Association for Computational Linguistics: System Demonstrations*, pages 25–30, Florence, Italy, 2019.

[21] Xiaoyuan Yi, Maosong Sun, Ruoyu Li, and Wenhao Li. Automatic poetry generation with mutual reinforcement learning. In *Proceedings of the 2018 Conference on Empirical Methods in Natural Language Processing*, pages 3143–3153, Brussels, Belgium, 2018.

[22] Huimin Chen, Xiaoyuan Yi, Maosong Sun, Cheng Yang, Wenhao Li, and Zhipeng Guo. Sentiment-controllable Chinese poetry generation. In *Proceedings of the Twenty-Eighth International Joint Conference on Artificial Intelligence*, Macao, China, 2019.

[23] Bo Zhou, Qianglong Chen, Tianyu Wang, Xiaomi Zhong, and Yin Zhang. WYWEB: A NLP evaluation benchmark for classical Chinese. In *Findings of the Association for Computational Linguistics: ACL 2023*, pages 3294–3319, Toronto, Canada, July 2023. Association for Computational Linguistics.

[24] Wenxuan Zhang, Mahani Aljunied, Chang Gao, Yew Ken Chia, and Lidong Bing. M3Exam: A multilingual, multimodal, multilevel benchmark for examining large language models. *Advances in Neural Information Processing Systems*, 36, 2023.

[25] Zexuan Qiu, Jingjing Li, Shijue Huang, Wanjun Zhong, and Irwin King. CLongEval: A Chinese benchmark for evaluating long-context large language models, 2024.

[26] Zhexin Zhang, Leqi Lei, Lindong Wu, Rui Sun, Yongkang Huang, Chong Long, Xiao Liu, Xuanyu Lei, Jie Tang, and Minlie Huang. SafetyBench: Evaluating the safety of large language models with multiple choice questions, 2023.

[27] Liang Xu, Hai Hu, Xuanwei Zhang, Lu Li, Chenjie Cao, Yudong Li, Yechen Xu, Kai Sun, Dian Yu, Cong Yu, et al. CLUE: A Chinese language understanding evaluation benchmark. In *Proceedings of the 28th International Conference on Computational Linguistics*, pages 4762–4772, Barcelona, Spain (Online), December 2020. International Committee on Computational Linguistics.

[28] Liang Xu, Anqi Li, Lei Zhu, Hang Xue, Changtai Zhu, Kangkang Zhao, Haonan He, Xuanwei Zhang, Qiyue Kang, and Zhenzhong Lan. SuperCLUE: A comprehensive Chinese large language model benchmark. *arXiv preprint arXiv:2307.15020*, 2023.

[29] Qinyuan Cheng, Tianxiang Sun, Wenwei Zhang, Siyin Wang, Xiangyang Liu, Mozhi Zhang, Junliang He, Mianqiu Huang, Zhangyue Yin, Kai Chen, and Xipeng Qiu. Evaluating hallucinations in Chinese large language models. *CoRR*, abs/2310.03368, 2023.

[30] Xiao Liu, Xuanyu Lei, Shengyuan Wang, Yue Huang, Zhuoer Feng, Bosi Wen, Jiale Cheng, Pei Ke, Yifan Xu, Weng Lam Tam, et al. AlignBench: Benchmarking Chinese alignment of large language models, 2023.

[31] Yufei Huang and Deyi Xiong. CBBQ: A Chinese bias benchmark dataset curated with human-AI collaboration for large language models. *arXiv preprint arXiv:2306.16244*, 2023.

[32] Zongyuan Jiang, Jiapeng Wang, Jiahuan Cao, Xue Gao, and Lianwen Jin. Towards better translations from classical to modern Chinese: A new dataset and a new method. In *CCF International Conference on Natural Language Processing and Chinese Computing*, pages 387–399. Springer, 2023.

[33] Daizhigev20. https://github.com/garychowcmu/daizhigev20.

[34] Tom Brown, Benjamin Mann, Nick Ryder, Melanie Subbiah, Jared D Kaplan, Prafulla Dhariwal, Arvind Neelakantan, Pranav Shyam, Girish Sastry, Amanda Askell, et al. Language models are few-shot learners. *Advances in neural information processing systems*, 33:1877–1901, 2020.

[35] Guilherme Penedo, Quentin Malartic, Daniel Hesslow, Ruxandra Cojocaru, Hamza Alobeidli, Alessandro Cappelli, Baptiste Pannier, Ebtesam Almazrouei, and Julien Launay. The Refined-Web dataset for Falcon LLM: Outperforming curated corpora with web data only. *Advances in Neural Information Processing Systems*, 36, 2024.

[36] Jiantao Qiu, Haijun Lv, Zhenjiang Jin, Rui Wang, Wenchang Ning, Jia Yu, ChaoBin Zhang, Pei Chu, Yuan Qu, Runyu Peng, et al. WanJuan-CC: A safe and high-quality open-sourced English webtext dataset. *arXiv preprint arXiv:2402.19282*, 2024.

[37] Sha Yuan, Hanyu Zhao, Zhengxiao Du, Ming Ding, Xiao Liu, Yukuo Cen, Xu Zou, Zhilin Yang, and Jie Tang. Wudaocorpora: A super large-scale Chinese corpora for pre-training language models. *AI Open*, 2:65–68, 2021.

[38] Josh Achiam, Steven Adler, Sandhini Agarwal, Lama Ahmad, Ilge Akkaya, Florencia Leoni Aleman, Diogo Almeida, Janko Altenschmidt, Sam Altman, Shyamal Anadkat, et al. GPT-4 technical report. *arXiv preprint arXiv:2303.08774*, 2023.

[39] Hugo Touvron, Thibaut Lavril, Gautier Izacard, Xavier Martinet, Marie-Anne Lachaux, Timothée Lacroix, Baptiste Rozière, Naman Goyal, Eric Hambro, Faisal Azhar, et al. LLaMA: Open and efficient foundation language models. *arXiv preprint arXiv:2302.13971*, 2023.

[40] Hugo Touvron, Louis Martin, Kevin Stone, Peter Albert, Amjad Almahairi, Yasmine Babaei, Nikolay Bashlykov, Soumya Batra, Prajjwal Bhargava, Shruti Bhosale, et al. LLaMA 2: Open foundation and fine-tuned chat models. *arXiv preprint arXiv:2307.09288*, 2023.

[41] Jinze Bai, Shuai Bai, Yunfei Chu, Zeyu Cui, Kai Dang, Xiaodong Deng, Yang Fan, Wenbin Ge, Yu Han, Fei Huang, et al. Qwen technical report. *arXiv preprint arXiv:2309.16609*, 2023.

[42] Aiyuan Yang, Bin Xiao, Bingning Wang, Borong Zhang, Ce Bian, Chao Yin, Chenxu Lv, Da Pan, Dian Wang, Dong Yan, et al. Baichuan 2: Open large-scale language models. *arXiv preprint arXiv:2309.10305*, 2023.

[43] Zhengxiao Du, Yujie Qian, Xiao Liu, Ming Ding, Jiezhong Qiu, Zhilin Yang, and Jie Tang. GLM: General language model pretraining with autoregressive blank infilling. In *Proceedings of the 60th Annual Meeting of the Association for Computational Linguistics*, 2022.

[44] Aohan Zeng, Xiao Liu, Zhengxiao Du, Zihan Wang, Hanyu Lai, Ming Ding, Zhuoyi Yang, Yifan Xu, Wendi Zheng, Xiao Xia, et al. GLM-130B: An open bilingual pre-trained model. In *The Eleventh International Conference on Learning Representations*, 2022.

[45] FlagAlpha. Llama2-Chinese-7b-Chat. https://huggingface.co/FlagAlpha/Llama2-Chinese-7b-Chat, 2024. Hugging Face.

[46] FlagAlpha. Llama2-Chinese-13b-Chat. https://huggingface.co/FlagAlpha/Llama2-Chinese-13b-Chat, 2024. Hugging Face.

[47] Xunzi-LLM of Chinese-classics. XunziALLM. https://github.com/Xunzi-LLM-of-Chinese-classics/XunziALLM, 2024. GitHub.

[48] Wptoux. Bloom-7B-Chunhua. https://huggingface.co/wptoux/bloom-7b-chunhua, 2024. Hugging Face.

[49] Zhe Zhang, Jie Liu, Lihua Chi, and Xinhai Chen. Word-level BERT-CNN-RNN model for Chinese punctuation restoration. In *2020 IEEE 6th International Conference on Computer and Communications (ICCC)*, pages 1629–1633. IEEE, 2020.

[50] Kishore Papineni, Salim Roukos, Todd Ward, and Wei-Jing Zhu. BLEU: A method for automatic evaluation of machine translation. In *Proceedings of the 40th annual meeting of the Association for Computational Linguistics*, pages 311–318, 2002.

[51] PeterH0323. Ancient-Chat-LLM. https://github.com/PeterH0323/ancient-chat-llm, 2024. GitHub.

[52] 01. AI, Alex Young, Bei Chen, Chao Li, Chengen Huang, Ge Zhang, Guanwei Zhang, Heng Li, Jiangcheng Zhu, Jianqun Chen, et al. Yi: Open foundation models by 01.AI, 2024.

[53] Jared Kaplan, Sam McCandlish, Tom Henighan, Tom B. Brown, Benjamin Chess, Rewon Child, Scott Gray, Alec Radford, Jeffrey Wu, and Dario Amodei. Scaling laws for neural language models, 2020.

[54] YeungNLP. Firefly-Baichuan2-13B. https://huggingface.co/YeungNLP/firefly-baichuan2-13b, 2024. Hugging Face.

[55] Zheng Cai, Maosong Cao, Haojiong Chen, Kai Chen, Keyu Chen, Xin Chen, Xun Chen, Zehui Chen, Zhi Chen, Pei Chu, et al. InternLM2 technical report, 2024.

[56] Iflytek. Spark-v3.5. https://xinghuo.xfyun.cn/.

[57] Google. Gemma-1.1-7B-IT. https://huggingface.co/google/gemma-1.1-7b-it, 2024. Hugging Face.

[58] IDEA-CCNL. Ziya-LLaMA-13B-v1.1. https://huggingface.co/IDEA-CCNL/Ziya-LLaMA-13B-v1.1.

[59] Meta. LLaMA3. https://ai.meta.com/blog/meta-llama-3.

[60] Shenzhi Wang. Llama3-8B-Chinese-Chat. https://huggingface.co/shenzhi-wang/Llama3-8B-Chinese-Chat, 2024. Hugging Face.

[61] OpenAI. GPT-3.5. https://openai.com/index/gpt-3-5-turbo-fine-tuning-and-api-updates.

[62] OpenAI, Josh Achiam, Steven Adler, Sandhini Agarwal, Lama Ahmad, Ilge Akkaya, Florencia Leoni Aleman, Diogo Almeida, Janko Altenschmidt, Sam Altman, et al. GPT-4 technical report, 2024.

[63] Yang Liu, Jiahuan Cao, Chongyu Liu, Kai Ding, and Lianwen Jin. Datasets for large language models: A comprehensive survey. *arXiv preprint arXiv:2402.18041*, 2024.

[64] Yupeng Chang, Xu Wang, Jindong Wang, Yuan Wu, Linyi Yang, Kaijie Zhu, Hao Chen, Xiaoyuan Yi, Cunxiang Wang, Yidong Wang, Wei Ye, Yue Zhang, Yi Chang, Philip S. Yu, Qiang Yang, and Xing Xie. A survey on evaluation of large language models. *ACM Trans. Intell. Syst. Technol.*, 15(3), mar 2024.

[65] Teven Le Scao, Angela Fan, Christopher Akiki, Ellie Pavlick, Suzana Ilić, Daniel Hesslow, Roman Castagné, Alexandra Sasha Luccioni, François Yvon, Matthias Gallé, et al. Bloom: A 176B-parameter open-access multilingual language model, 2023.

[66] Yushi Bai, Jiahao Ying, Yixin Cao, Xin Lv, Yuze He, Xiaozhi Wang, Jifan Yu, Kaisheng Zeng, Yijia Xiao, Haozhe Lyu, et al. Benchmarking foundation models with language-model-as-an-examiner. *Advances in Neural Information Processing Systems*, 36, 2023.

[67] Qinyuan Cheng, Tianxiang Sun, Wenwei Zhang, Siyin Wang, Xiangyang Liu, Mozhi Zhang, Junliang He, Mianqiu Huang, Zhangyue Yin, Kai Chen, et al. Evaluating hallucinations in Chinese large language models. *arXiv preprint arXiv:2310.03368*, 2023.

[68] Yang Liu, Dan Iter, Yichong Xu, Shuohang Wang, Ruochen Xu, and Chenguang Zhu. G-eval: NLG evaluation using GPT-4 with better human alignment. In Houda Bouamor, Juan Pino, and Kalika Bali, editors, *Proceedings of the 2023 Conference on Empirical Methods in Natural Language Processing*, pages 2511–2522, Singapore, December 2023. Association for Computational Linguistics.

[69] Lianmin Zheng, Wei-Lin Chiang, Ying Sheng, Siyuan Zhuang, Zhanghao Wu, Yonghao Zhuang, Zi Lin, Zhuohan Li, Dacheng Li, Eric Xing, et al. Judging LLM-as-a-judge with MT-Bench and Chatbot Arena. *Advances in Neural Information Processing Systems*, 36, 2023.

[70] Silvia Stopponi, Saskia Peels-Matthey, and Malvina Nissim. AGREE: A new benchmark for the evaluation of distributional semantic models of ancient Greek. *Digital Scholarship in the Humanities*, 39(1):373–392, 2024.

[71] Danlu Chen, Freda Shi, Aditi Agarwal, Jacobo Myerston, and Taylor Berg-Kirkpatrick. LogogramNLP: Comparing visual and textual representations of ancient logographic writing systems for NLP. In *Proceedings of the 62nd Annual Meeting of the Association for Computational Linguistics (Volume 1: Long Papers)*, pages 14238–14254, 2024.

[72] Amrith Krishna, Pavankumar Satuluri, and Pawan Goyal. A dataset for Sanskrit word segmentation. In *Proceedings of the Joint SIGHUM Workshop on Computational Linguistics for Cultural Heritage, Social Sciences, Humanities and Literature*, pages 105–114, 2017.

[73] Ann Taylor. The York-Toronto-Helsinki parsed corpus of old English prose. In *Creating and digitizing language corpora: Volume 2: Diachronic Databases*, pages 196–227. Springer, 2003.

# A Datasheet for WenMind

## A.1 Motivation

**1. For what purpose was the dataset created? Was there a specific task in mind? Was there a specific gap that needed to be filled? Please provide a description.**

A1: The purpose of WenMind is to quantify the capabilities of existing large language models (LLMs) on Chinese classical literature and language arts (CCLLA), providing a reference for the development of this field. Existing benchmarks focus primarily on the Ancient Prose sub-domain or contain only a limited number of tasks, making it challenging to provide a thorough and holistic assessment of LLMs' capabilities in CCLLA. To fill this gap, we propose WenMind, a comprehensive benchmark dedicated for evaluating LLMs in CCLLA. WenMind covers the sub-domains of Ancient Prose, Ancient Poetry, and Ancient Literary Culture, comprising 4,875 question-answer pairs, spanning 42 fine-grained tasks, 3 question formats, and 2 evaluation scenarios: domain-oriented and capability-oriented. Overall, WenMind serves as a standardized and comprehensive baseline, providing valuable insights for future CCLLA research.

**2. Who created this dataset (e.g., which team, research group) and on behalf of which entity (e.g., company, institution, organization)?**

A2: The WenMind benchmark is created by the Deep Learning and Vision Computing Lab (DLVC-Lab) of South China University of Technology, INTSIG Information Co., Ltd, and INTSIG-SCUT Joint Lab on Document Analysis and Recognition.

## A.2 Composition

**1. What do the instances that comprise the dataset represent (e.g., documents, photos, people, countries)? Are there multiple types of instances (e.g., movies, users, and ratings; people and interactions between them; nodes and edges)? Please provide a description.**

A1: The Wenmind benchmark consists of 4,875 plain text questions. Each question is represented in a dictionary format, containing the following keys: ID, domain, capability, question format, coarse-grained task, fine-grained task, question, and answer. The dataset is entirely stored in a JSON file.

**2. How many instances are there in total (of each type, if appropriate)?**

A2: The WenMind benchmark comprises a total of 4,875 instances, which are categorized into three distinct sections: 1,900 instances in the Ancient Prose section, 1,845 instances in the Ancient Poetry section, and 1,130 instances in the Ancient Literature Culture section.

**3. Does the dataset contain all possible instances or is it a sample (not necessarily random) of instances from a larger set? If the dataset is a sample, then what is the larger set? Is the sample representative of the larger set (e.g., geographic coverage)? If so, please describe how this represent ativeness was validated/verified. If it is not representative of the larger set, please describe why not (e.g., to cover a more diverse range of instances because instances were withheld or unavailable).**

A3: The WenMind benchmark encompasses all possible instances.

**4. What data does each instance consist of?**

A4: The WenMind benchmark consists of textual data. Each text instance includes the following information: ID, domain, capability, question format, coarse-grained task, fine-grained task, question, and answer.

**5. Is there a label or target associated with each instance?**

A5: The WenMind benchmark includes questions and their corresponding answers, along with additional information such as the question format and task name.

**6. Is any information missing from individual instances? If so, please provide a description, explaining why this information is missing (e.g., because it was unavailable). This does not include intentionally removed information, but might include, e.g., redacted text.**

A6: No, each of our data instances contains comprehensive information.

**7. Are relationships between individual instances made explicit (e.g., users' movie ratings, social network links)? If so, please describe how these relationships are made explicit.**

A7: No, our data instances are independent of each other and have no interrelations.

**8. Are there recommended data splits (e.g., training, development/validation, testing)? If so, please provide a description of these splits, explaining the rationale behind them.**

A8: No, our data is utilized solely for the purpose of testing and not for training.

**9. Are there any errors, sources of noise, or redundancies in the dataset? If so, please provide a description.**

A9: In WenMind, we have incorporated extensive manual calibration to minimize errors and noise as much as possible, yet we cannot guarantee that every answer will be flawless.

**10. Is the dataset self-contained, or does it link to or otherwise rely on external resources (e.g., websites, tweets, other datasets)?**

A10: Yes, the WenMind benchmark is self-contained.

**11. Does the dataset contain data that might be considered confidential (e.g., data that is protected by legal privilege or by doctorpatient confidentiality, data that includes the content of individuals non-public communications)? If so, please provide a description.**

A11: No, we construct the WenMind benchmark from publicly accessible sources.

**12. Does the dataset contain data that, if viewed directly, might be offensive, insulting, threatening, or might otherwise cause anxiety? If so, please describe why.**

A12: No, the WenMind benchmark does not contain any content that is offensive, insulting, threatening, or might otherwise cause anxiety.

**13. Does the dataset relate to people?**

A13: No, the WenMind benchmark has nothing to do with people.

## A.3 Collection Process

**1. How was the data associated with each instance acquired? Was the data directly observable (e.g., raw text, movie ratings), reported by subjects (e.g., survey responses), or indirectly inferred/derived from other data (e.g., part-of-speech tags, model-based guesses for age or language)? If data was reported by subjects or indirectly inferred/derived from other data, was the data validated/verified? If so, please describe how.**

A1: The data construction and specific sources are thoroughly described in Section 3.2 and Appendix B.3. The text we gather from the Internet is sourced from several open copyright websites, such as `https://www.sou-yun.cn/` and `https://www.zdic.net/`.

**2. What mechanisms or procedures were used to collect the data (e.g., hardware apparatus or sensor, manual human curation, software program, software API)? How were these mechanisms or procedures validated?**

A2: Our data collection is carried out through three mechanisms: manual human curation, a web crawling program, and the API of LLMs.

**3. Who was involved in the data collection process (e.g., students, crowdworkers, contractors) and how were they compensated (e.g., how much were crowdworkers paid)**

A3: The WenMind benchmark is created by the researchers from Deep Learning and Vision Computing Lab (DLVC-Lab) of South China University of Technology, INTSIG Information Co., Ltd, and INTSIG-SCUT Joint Lab on Document Analysis and Recognition.

**4. Over what timeframe was the data collected?**

A4: Our data collection spans a period of two months.

**5. Did you collect the data from the individuals in question directly, or obtain it via third parties or other sources (e.g., websites)?**

A5: We mainly collect the data from the Internet, open-source datasets, and LLM generation.

### A.4 Preprocessing/cleaning/labeling

**1. Was any preprocessing/cleaning/labeling of the data done (e.g., discretization or bucketing, tokenization, part-of-speech tagging, SIFT feature extraction, removal of instances, processing of missing values)? If so, please provide a description. If not, you may skip the remaining questions in this section.**

A1: Yes, the detail is discussed in Section 3.2.

**2. Was the "raw" data saved in addition to the preprocessed/cleaned/labeled data (e.g., to support unanticipated future uses)? If so, please provide a link or other access point to the "raw" data.**

A2: No, the data we provide has undergone processing and transformation, combining original texts with annotations for storage.

### A.5 Uses

**1. Has the dataset been used for any tasks already? If so, please provide a description.**

A1: No, the WenMind benchmark has not appeared in previous works.

**2. What (other) tasks could the dataset be used for?**

A2: No, the WenMind benchmark is solely used for the evaluation of LLMs.

**3. Are there tasks for which the dataset should not be used? If so, please provide a description.**

A3: To prevent data leakage and ensure the fairness of the evaluation [63, 64], the WenMind benchmark cannot be used for training LLMs.

### A.6 Distribution

**1. How will the dataset will be distributed (e.g., tarball on website, API, GitHub)?**

A1: Our dataset is publicly available at https://github.com/SCUT-DLVCLab/WenMind.

**2. When will the dataset be distributed?**

A2: Our dataset has been already distributed.

**3. Will the dataset be distributed under a copyright or other intellectual property (IP) license, and/or under applicable terms of use (ToU)?**

A3: For details, please refer to Appendix F.

### A.7 Maintenance

**1. Who will be supporting/hosting/maintaining the dataset?**

A1: The Deep Learning and Vision Computing Lab (DLVC-Lab) of South China University of Technology.

**2. How can the owner/curator/manager of the dataset be contacted (e.g., email address)?**

A2: The manager can be contacted through the email address or Github.

**3. Will the dataset be updated (e.g., to correct labeling errors, add new instances, delete instances)? If so, please describe how often, by whom, and how updates will be communicated to dataset consumers (e.g., mailing list, GitHub)?**

A3: We will publish a correction list for the WenMind benchmark on GitHub every quarter.

**4. Will older versions of the dataset continue to be supported/hosted/maintained? If so, please describe how. If not, please describe how its obsolescence will be communicated to dataset consumers.**

A4: We will maintain all versions of the WenMind benchmark on GitHub.

**5. If others want to extend/augment/build on/contribute to the dataset, is there a mechanism for them to do so? If so, please provide a description. Will these contributions be validated/verified? If so, please describe how. If not, why not? Is there a process for communicating/distributing these contributions to dataset consumers? If so, please provide a description.**

A5: If other researchers or individuals are interested in extending, augmenting, building on, or contributing to the dataset, they should contact us via email, clearly articulating their intentions and requesting our consent prior to any further actions.

# B Dataset

## B.1 Task Description

Table 6 provides detailed information for 42 fine-grained tasks, including the Chinese and English task names, task descriptions, assessed capabilities, scale, average question length, average answer length, and question format.

Table 6: Detail of each task in WenMind. "Q" represents "Question"; "A" represents "Answer"; "FB" represents "Fill-in-the-Blank questions"; "MCQ" represents "Multiple-Choice Questions"; "QA" represents "Question-and-Answer questions". Zoom in for better view.

| ID | Task Name | Task Description | Capability | Scale | Avg.Q Tokens | Avg.A Tokens | FB | MCQ | QA |
|---|---|---|---|---|---|---|---|---|---|
| T1-1 | 倒装句语序<br>Inverted Sentence Structure | Correct word order for inverted sentences | Understanding | 18 | 22.22 | 8.89 | | | ✔ |
| T1-2 | 省略句<br>Elliptical Sentence | Answer the omitted information in the elliptical sentence | Understanding | 32 | 34.88 | 33.25 | | | ✔ |
| T1-3 | 倒装句类型<br>Inverted Sentence Types | Identify the inversion type of inverted sentences | Understanding | 7 | 55.43 | 1.00 | | ✔ | |
| T1-4 | 判断句式<br>Sentence Structure Identification | Identify the sentence's syntactic type | Understanding | 43 | 46.33 | 1.00 | | ✔ | |
| T2 | 文白翻译<br>Classical Chinese to Modern Chinese | Translate classical Chinese into modern Chinese | Understanding | 200 | 31.91 | 30.74 | | | ✔ |
| T3 | 白文翻译<br>Modern Chinese to Classical Chinese | Translate modern Chinese into classical Chinese | Understanding | 200 | 42.46 | 19.56 | | | ✔ |
| T4 | 命名实体识别<br>Named Entity Recognition | Extract named entities from ancient prose sentences | Understanding | 200 | 236.99 | 43.20 | | | ✔ |
| T5 | 句读<br>Punctuation | Add punctuation to ancient prose sentences | Understanding | 200 | 34.95 | 26.50 | | | ✔ |
| T6 | 主题分类<br>Topic Classification | Select theme categories based on ancient prose sentences | Understanding | 200 | 172.40 | 2.00 | | | ✔ |
| T7 | 字词解释<br>Word Explanation | Explain the words and phrases in ancient prose sentences | Understanding | 100 | 32.84 | 27.21 | | | ✔ |
| T8 | 阅读理解<br>Reading Comprehension | Read ancient prose texts and answer related questions | Understanding | 100 | 207.04 | 81.47 | | | ✔ |
| T9 | 虚词<br>Function Words | Answer the usage of function words in ancient prose sentences | Understanding | 100 | 76.88 | 1.00 | | ✔ | |
| T10 | 通假字<br>Homophones | Identify whether a character is a homophone | Understanding | 200 | 145.94 | 1.00 | | ✔ | |
| T11 | 单字多义<br>Polysemy | Distinguish between different meanings of the same character | Understanding | 200 | 127.40 | 1.00 | | ✔ | |
| T12 | 文言文写作<br>Ancient Prose Writing | Writing in classical Chinese | Generation | 100 | 102.96 | 809.96 | | | ✔ |
| T13-1 | 赏析真题<br>Appreciation Exam Questions | Answer appreciation questions based on ancient poetry | Understanding | 150 | 200.69 | 68.42 | | ✔ | ✔ |
| T13-2 | 自由赏析<br>Free Appreciation | Conduct a free and detailed analysis of ancient poetry | Understanding | 100 | 112.24 | 189.24 | | | ✔ |
| T14-1 | 诗创作<br>Poetry Writing | Compose a poem based on the theme | Generation | 30 | 18.97 | 59.83 | | | ✔ |
| T14-2 | 词创作<br>Ci Writing | Compose a Ci based on the theme | Generation | 50 | 25.14 | 84.98 | | | ✔ |
| T14-3 | 曲创作<br>Qu Writing | Compose a Qu based on the theme | Generation | 20 | 25.15 | 54.40 | | | ✔ |
| T15-1 | 内容问答<br>Content Q&A | Answer the complete content of ancient poetry according to the title and author | Knowledge | 200 | 29.62 | 59.88 | | | ✔ |
| T15-2 | 题目作者问答<br>Title and Author Q&A | Answer the title and author according to the content of ancient poetry | Knowledge | 200 | 85.25 | 15.76 | | | ✔ |
| T15-3 | 下句默写<br>Write the Next Sentence | Write the next sentence according to the previous sentence in the ancient poem | Knowledge | 100 | 34.67 | 6.36 | | | ✔ |
| T15-4 | 上句默写<br>Write the Previous Sentence | Write the previous sentence according to the next sentence in the ancient poem | Knowledge | 100 | 35.83 | 6.24 | | | ✔ |
| T15-5 | 理解性默写<br>Comprehension Dictation | Provide ancient poetry sentences that meet the requirements | Knowledge | 30 | 67.93 | 14.93 | ✔ | | |
| T15-6 | 判断体裁<br>Genre Judgment | Judge the genre of ancient poetry | Knowledge | 120 | 87.53 | 3.25 | | | ✔ |
| T16 | 古诗词翻译<br>Ancient Poetry Translation | Translate ancient poetry into modern Chinese | Understanding | 200 | 80.71 | 104.61 | | | ✔ |
| T17 | 情感分类<br>Sentiment Classification | Judge the sentiment contained in ancient poetry | Understanding | 200 | 111.16 | 3.28 | | ✔ | |
| T18 | 古诗词英文翻译<br>Ancient Poetry to English | Translate ancient poetry into English | Understanding | 50 | 73.24 | 392.98 | | | ✔ |
| T19 | 诗人介绍<br>Poet Introduction | Provide a detailed introduction of the poet | Knowledge | 110 | 11.53 | 346.51 | | | ✔ |
| T20 | 意象解析<br>Analysis of Imagery | Provide the meanings of the imagery | Knowledge | 185 | 32.03 | 228.39 | | | ✔ |
| T21-1 | 接下联<br>Couplet Following | Create the following couplet based on the previous one | Generation | 100 | 19.92 | 9.92 | | | ✔ |
| T21-2 | 对联创作<br>Couplet Writing | Write a couplet based on the theme | Generation | 100 | 16.48 | 18.78 | | | ✔ |
| T21-3 | 拟横批<br>HengPi Writing | Write HengPi based on the content of a couplet | Generation | 100 | 28.80 | 4.00 | | | ✔ |
| T22-1 | 近义词<br>Synonyms | Provide the synonym for the idiom | Knowledge | 100 | 16.19 | 6.40 | | | ✔ |
| T22-2 | 成语出处<br>The Origin of Idiom | Provide the source of the idiom | Knowledge | 100 | 13.33 | 41.40 | | | ✔ |
| T22-3 | 成语蕴含<br>Idiom Finding | Extract idioms from ancient Chinese sentences and provide their meanings | Knowledge | 100 | 66.45 | 30.10 | | | ✔ |
| T22-4 | 成语解释<br>Idiom Explanation | Provide the meaning of idioms | Knowledge | 100 | 15.14 | 28.46 | | | ✔ |
| T23 | 谜语<br>Riddle | Guess the answer based on clues or clever hints | Knowledge | 100 | 17.94 | 2.14 | | | ✔ |
| T24 | 歇后语<br>Xiehouyu | Complete the second half of the proverb based on the first half | Knowledge | 100 | 16.52 | 5.06 | | | ✔ |
| T25 | 古汉语音韵<br>Historical Chinese Phonology | Answer questions about ancient Chinese phonetics and rhymes | Knowledge | 100 | 61.99 | 1.00 | | ✔ | |
| T26 | 国学常识问答<br>Knowledge of Sinology Q&A | Answer questions about Sinology | Knowledge | 130 | 22.12 | 10.69 | | ✔ | ✔ |

## B.2 Task Example

Figure 7, 8, 9, 10, 11 and 12 respectively present examples of question-answer pairs for the 42 fine-grained tasks.



**T1-1 倒装句语序 Inverted Sentence Structure**

**Question:**
微斯人，吾谁与归？ （Inverted sentence）
上面倒装句的正确语序是： The correct word order of the inverted sentence above is:

**Answer:**
微斯人，吾与谁归？（The sentence with the correct word order）

**T1-2 省略句 Elliptical Sentence**

**Question:**
《小石潭记》中"日光下澈潭水"是一句省略句，该句子省略了什么？
In the classical Chinese essay "小石潭记", "日光下澈潭水" is an elliptical sentence. What is omitted in this sentence?

**Answer:**
该句子省略了"于"，完整表达为"日光下澈（于）潭水"。
The sentence omitted "于" and the complete expression is "日光下澈（于）潭水".

**T1-3 倒装句类型 Inverted Sentence Types**

**Question:**
其两膝相比者。（Inverted sentence）
上面倒装句的倒装类型是： The inversion type of the inverted sentence above is:
A、主谓倒装 B、宾语前置 C、定语后置 D、介词结构后置 （Options）

**Answer: C**

**T1-4 判断句式 Sentence Structure Identification**

**Question:**
吾局促一室之内,欲出不得。（Classical Chinese sentence）
上面句子的句式是： The sentence structure of the above sentence is:
A、省略句 B、判断句 C、被动句 D、倒装句 （Options）

**Answer: D**

**T2 文白翻译 Classical Chinese to Modern Chinese**

**Question:**
把文言文翻译为白话文： Translate classical Chinese into modern Chinese:
问曰。坏塔好不。坏房好不。（Classical Chinese sentence）

**Answer:**
问他们对破坏塔寺及破坏僧舍有何意见。（Modern Chinese sentence）

**T3 白文翻译 Modern Chinese to Classical Chinese**

**Question:**
把白话文翻译为文言文： Translate modern Chinese into classical Chinese:
他人应来拜访您。（Modern Chinese sentence）

**Answer:**
彼应诣此。（Classical Chinese sentence）

**T4 命名实体识别 Named Entity Recognition**

**Question:**
抽取下列文言文句子中的命名实体： Extract named entities from the following classical Chinese sentence:
秦都咸阳，在府西微北四十里，本杜县地也。（Classical Chinese sentence）

**Answer:**
秦，咸阳，杜县 （Named entities）

Figure 7: Task 1-1 to Task 4 examples.

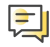 **T5 句读** *Punctuation*

**Question:**
给下面句子添加标点符号：*Add punctuation to the following sentence:*
其下才者模拓旧文回头易尾或有相呈新制见模拓之文为之愧赧 *(Classical Chinese sentence)*

**Answer:**
其下才者，模拓旧文，回头易尾，或有相呈新制，见模拓之文，为之愧赧。
*(Classical Chinese sentence with punctuation added)*

---

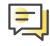 **T6 主题分类** *Topic Classification*

**Question:**
类别：佛藏、史藏、诗藏、集藏、子藏、易藏、艺藏、医藏、道藏、儒藏。 *(Categories)*
下面句子的主题属于上述哪一类：*What category does the topic of the following sentence belong to:*
凡消渴病少愈，不亟回枯泽槁，听其土燥不生，致酿疮痏无救，医之罪也。*(Classical Chiese sentence)*

**Answer:**
医藏 *(Category)*

---

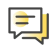 **T7 字词解释** *Word Explanation*

**Question:**
请问在文言文句子"其有智者见利倍多，便易其丝为于疋帛"中，"易"字的含义是什么？
*What is the meaning of the character "易" in the classical Chinese sentence '其有智者见利倍多，便易其丝为于疋帛'?*

**Answer:**
在这里，"易"字的意思是"交换"，指用一种东西换取另一种东西。
*Here, the character "易" means "exchange", referring to trading one thing for another.*

---

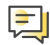 **T8 阅读理解** *Reading Comprehension*

**Question:**
请根据文言文回答问题。*Please answer the question based on classical Chinese.*
温彦博为吏部侍郎，有选人裴略被放...... *(Classical Chinese sentences)*
裴略通过什么方式最终获得了官职？*How did Pei Lue finally obtain an official position?*

**Answer:**
裴略通过自赞并展示自己的才华，用尖刻的嘲讽回应温彦博的考验，最终使温彦博感到惭愧，从而获得了官职。
*Pei Lue responded to Wen Yanbo's test with a sharp mockery by praising himself and showing his talent, which ultimately made Wen Yanbo feel ashamed and earned him an official position.*

---

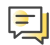 **T9 虚词** *Function Words*

**Question:**
并自为其名。*(Classical Chinese sentence)*
上面句子中"其"的用法是：*The usage of "其" in the above sentence is:*
A、第三人称代词 B、第一人称代词 C、表示反问语气 D、指示代词 *(Options)*

**Answer: B**

---

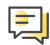 **T10 通假字** *Homophones*

**Question:**
下列选项中[]内的"大"字是通假字的是：
*In the following options, which of the "大" in the brackets [] is a homophone:*
A、遵彼[大]路兮。 B、[大]宰，卿一人。C、我[大]烧烛夜香咱。D、其朝践用两[大]尊。*(Options)*

**Answer: C**

---

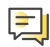 **T11 单字多义** *Polysemy*

**Question:**
下列选项中[]内的"与"字翻译为"同类，同盟者"的是
*The character "与" in the brackets [] in the following options is translated as "similar, ally" is*
A、我能为君约 [与] 国，战必克。 B、鸟兽不可 [与] 同群。
C、[与] 武昌革命之役并寿。　　　D、匈奴壤界兽圈，孤弱无 [与]，此困亡之时也。*(Options)*

**Answer: A**

Figure 8: Task 5 to Task 11 examples.

**T12 文言文写作 Ancient Prose Writing**

**Question:**
帮我写一个减肥训练营的招生文案，不减下来不收钱。用文言文写。
Please help me write an enrollment advertisement for a weight loss camp, where no fees are charged if the weight is not lost, in classical Chinese.

**Answer:**
入我减肥营门，向赘肉告别！吾辈誓言："不瘦，则不金！"盖信唯有真实之功，堪匹君子之汗与投……(The content written in classical Chinese)

---

**T13-1 赏析真题 Appreciation Exam Questions**

**Question:**
南乡子 唐·李王旬
烟漠漠，雨凄凄，岸花零落鹧鸪啼。远客扁舟临野渡，思乡处，潮退水平春色暮。(Ancient poem content)
这首词所要表现的主旨是？ What is the main theme this poem aims to convey?

**Answer:**
离愁别恨（或思乡之愁） Parting sorrow and separation grief (or homesickness)

---

**T13-2 自由赏析 Free Appreciation**

**Question:**
题目：燕归梁·春愁 作者：石延年
内容：芳草年年惹恨幽。想前事悠悠。伤春伤别几时休。算从古、为风流。春山总把，深匀翠黛，千叠在眉头。不知供得几多愁。更斜日、凭危楼。(Ancient poem content)
请对这首古诗词进行自由赏析。 Please provide a free appreciation of this ancient poem.

**Answer:**
此词题为春愁，写得蕴藉深情，表现出女主人公执着纯真的情感。
The poem is titled Spring Sorrow, which is written with deep and affectionate expression, showing the heroine's persistent and innocent emotions.

---

**T14-1 诗创作 Poetry Writing**

**Question:**
请创作一首以"夏日"为主题的五言绝句。 Please create a five-character quatrain with the theme of "summer".

**Answer:**
夏日
夏日暑气长，清风送晚凉。荷香盈水岸，蛙语闹池塘。(Ancient poem content)

---

**T14-2 词创作 Ci Writing**

**Question:**
请创作一首以"琴瑟"为主题的词（词牌名：如梦令）。
Please create a ci with the theme of "zither and lute", using the ci title "如梦令".

**Answer:**
如梦令·琴瑟和鸣
月下琴音瑟瑟，弦上情深意切。和鸣共此生，一曲相思难解。休歇，休歇，莫负良宵清绝。(Ci content)

---

**T14-3 曲创作 Qu Writing**

**Question:**
请创作一首以"人工智能"为主题的曲（曲牌名：清江引）。
Please create a qu with the theme of "artificial intelligence", using the qu title "清江引".

**Answer:**
清江引·人工智能
智能机器声声脆，巧手能言会。思维似海深，运算如飞电，人间万象皆能解。(Qu content)

---

**T15-1 内容问答 Content Q&A**

**Question:**
请给出诗人王安石（宋）的作品《登飞来峰》的完整内容。
Please provide the full text of the poem "Ascending Feilai Peak" by the poet Wang Anshi (Song Dynasty).

**Answer:**
内容：飞来山上千寻塔，闻说鸡鸣见日升。不畏浮云遮望眼，自缘身在最高层。(Ancient poem content)

Figure 9: Task 12 to Task 15-1 examples.

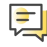 **T15-2 题目作者问答** *Title and Author Q&A*

**Question:**
根据下面的古诗词内容，给出相应的题目和作者。
*Based on the content of the ancient poem provided below, provide the corresponding title and author.*
古诗词内容：楼倚霜树外，镜天无一毫。南山与秋色，气势两相高。*(Ancient poem content)*

**Answer:**
题目：长安秋望 作者：杜牧 *(The title and author)*

---

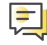 **T15-3 下句默写** *Write the Next Sentence*

**Question:**
李白写的《长干行》中，古诗词"郎骑竹马来"的下一句是什么？
*The next sentence after the ancient poetry phrase "郎骑竹马来" written by Li Bai in "长干行" is what?*

**Answer:**
绕床弄青梅 *(The content of the next sentence)*

---

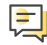 **T15-4 上句默写** *Write the Previous Sentence*

**Question:**
王维写的《杂诗》中，古诗词"应知故乡事"的上一句是什么？
*The previous sentence before the ancient poetry phrase "应知故乡事" written by Wang Wei in "杂诗" is what?*

**Answer:**
君自故乡来 *(The content of the previous sentence)*

---

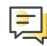 **T15-5 理解性默写** *Comprehension Dictation*

**Question:**
《蜀道难》的主旨句是：＿＿＿＿＿＿＿，＿＿＿＿＿＿＿
*The main theme sentence of "蜀道难" is ＿＿＿＿＿＿＿, ＿＿＿＿＿＿＿*

**Answer:**
蜀道之难，难于上青天。*(The main theme sentence)*

---

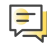 **T15-6 判断体裁** *Genre Judgment*

**Question:**
古诗词内容：赏得秦王镜，判不惜千金。非关欲照胆，特是自明心。*(Ancient poem content)*
请问这首古诗词的体裁是什么？ *What is the genre of this ancient Chinese poem?*

**Answer:**
五言绝句 *Five character quatrain*

---

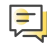 **T16 古诗词翻译** *Ancient Poetry Translation*

**Question:**
请将下面的古诗词内容翻译成白话文。*Please translate the following classical Chinese poem into modern Chinese.*
古诗词内容：岁晚风霜恶，将如遥夜何。残灯挑更暗，寒犬吠偏多。支枕成孤咏，怀人起浩歌。梅花不解饮，谁伴醉颜酡。*(Ancient poem content)*

**Answer:**
一年的风霜恶劣，将如此漫长的夜晚又能如何呢？灯残点燃尽昏暗难眠，寒犬吠声偏偏又多。孤枕难眠，孤零的旅途决然如歌，怀念的人心中怀念着心中的激起的愁绪。梅花不懂饮酒，有谁陪我喝酒，在颜色醉颜的我？*(Modern Chinese sentence)*

---

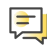 **T17 情感分类** *Sentiment Classification*

**Question:**
内容：漫山白白与红红，小树低丛看不供。总道楂花香气好，就中偏爱玉珑松。*(Ancient poem content)*
请从"负面"、"隐含负面"、"中性"、"隐含正面"、"正面"（负面程度由大到小）五项中选择最符合上述古诗词情感的一项。
*Please select the most suitable option among "negative", "implicitly negative", "neutral", "implicitly positive", and "positive" (with negative emotions ranging from strong to weak) to describe the sentiment of the above ancient poem.*

**Answer:**
正面 *(Positive)*

Figure 10: Task 15-2 to Task 17 examples.

 **T18 古诗词英文翻译 Ancient Poetry to English**

**Question:**
请将下面的古诗词内容翻译成英文：Please translate the content of the following ancient poem into English:
帘外雨潺潺，春意阑珊。(Ancient poem content)

**Answer:**
Rain is pitterpattering outside the window. Springtime spirit is on the wane.

---

 **T19 诗人介绍 Poet Introduction**

**Question:**
请介绍一下诗人"苏轼"。Please introduce the poet "Su Shi".

**Answer:**
苏轼（1037年—1101年），字子瞻，又字和仲，号铁冠道人、东坡居士，世称苏东坡、苏仙、坡仙……
(Information about Su Shi)

---

 **T20 意象解析 Analysis of Imagery**

**Question:**
"青罗"会作为意象出现在古诗词/古文中，该意象通常具有哪些含义？
"青罗" appears as an image in ancient Chinese poetry and prose. What are the common meanings associated with this image?

**Answer:**
(1) 青色丝织物。(2) 喻曲折环绕的碧水。
(1) Blue silk fabric. (2) A metaphor for the winding and clear blue water.

---

 **T21-1 接下联 Couplet Following**

**Question:**
根据上联对出下联：Match the couplet according to the first line:
柳色供诗莺声送酒 (The first line content)

**Answer:**
山光如画花气浸衣 (The second line content)

---

 **T21-2 主题创作 Couplet Writing**

**Question:**
给下面节日创作一幅对联：元宵节
Creating a couplet for the following festival: Lantern Festival

**Answer:**
宝地春醺风浩荡 瑶池夜宴玉玲珑 (The content of the couplet)

---

 **T21-3 拟横批 HengPi Writing**

**Question:**
给下面对联拟定一个横批：Create a HengPi for the following couplet:
福门家业腾云起 宝地财源乘风来 (The content of the couplet)

**Answer:**
五福临门 (The content of the HengPi)

---

 **T22-1 近义词 Synonyms**

**Question:**
给出下面成语的近义词：Provide a synonym for the following idiom:
足高气扬 (Idiom)

**Answer:**
志高气扬、趾高气扬 (Synonyms)

Figure 11: Task 18 to Task 22-1 examples.

 **T22-2 成语出处** *The Origin of Idiom*

**Question:**
下面成语出自哪里？指日而待
*Where does the idiom "指日而待" originate from?*

**Answer:**
清·褚人获《隋唐演义》第七十一回："将来执掌昭阳，可指日而待，为何夫人双眉反蹙起来？"
*(The origin of idiom)*

 **T22-3 成语蕴含** *Idiom Finding*
**Question:**
唐·王勃《秋日登洪府滕王阁饯别序》："闲云潭影日悠悠，物转星移几度秋。" *(Ancient poem content)*
从上面的内容中找出蕴含其中的成语，并给出成语的含义。
*From the content above, find the idiom hidden within and explain its meaning.*

**Answer:**
成语：斗转星移 含义：北斗转换了方向；星辰移了位置。形容时间的流逝；岁月的变迁。
*(The idiom and its meaning)*

 **T22-4 解释含义** *Idiom Explanation*

**Question:**
解释下面成语的意思：磐石之固
*Explain the meaning of the following idiom: 磐石之固 (Idiom)*

**Answer:**
比喻坚固不可动摇。*Metaphor for being firm and unshakable.*

 **T23 谜语** *Riddle*

**Question:**
猜谜语：萤火虫，亮晶晶（猜中国一城市）
*Guess the riddle: Fireflies, sparkling brightly (Guess a city in China)*

**Answer:**
昆明 *Kunming(Because "Kunming" literally means "bright insect" in Chinese)*

 **T24 歇后语** *Xiehouyu*

**Question:**
补全下列歇后语：*Complete the following Xiehouyu:*
十五只吊桶打水—— *(The first half of the Xiehouyu)*

**Answer:**
七上八下 *(The second half of the Xiehouyu)*

 **T25 古汉语音韵** *Historical Chinese Phonology*

**Question:**
下列哪个不是古声母的分系名称？*Which of the following is not a classification name of ancient initials?*
A、端系 B、知系 C、帮系 D、及系 *(Options)*

**Answer: D**

 **T26 国学常识问答** *Knowledge of Sinology Q&A*

**Question:**
古代最早的记事详备的编年体史书是？
*What is the earliest detailed and well-recorded annals history book in ancient times?*

**Answer:**
《左传》*Zuo Zhuan*

Figure 12: Task 22-2 to Task 26 examples.

Table 7: Data source information in WenMind. For details on M1-M5, please refer to Appendix B.3. Zoom in for better view.

| ID | Task Name | Data Collection Methods | | | | | Data Source | Related Link | License |
|---|---|---|---|---|---|---|---|---|---|
| | | M1 | M2 | M3 | M4 | M5 | | | |
| T1-1 | 倒装句语序<br>Inverted Sentence Structure | ✔ | | | | | Internet | - | - |
| T1-2 | 省略句<br>Elliptical Sentence | ✔ | | | | | Internet | - | - |
| T1-3 | 倒装句类型<br>Inverted Sentence Types | ✔ | | | | | Internet | - | - |
| T1-4 | 判断句式<br>Sentence Structure Identification | ✔ | | | | | Internet | - | - |
| T2 | 文白翻译<br>Classical Chinese to Modern Chinese | | | ✔ | | | C2MChn [32] | https://github.com/Zongyuan-Jiang/C2MChn | CC BY-NC-ND-4.0 |
| T3 | 白文翻译<br>Modern Chinese to Classical Chinese | | | ✔ | | | C2MChn [32] | https://github.com/Zongyuan-Jiang/C2MChn | CC BY-NC-ND-4.0 |
| T4 | 命名实体识别<br>Named Entity Recognition | | | | ✔ | | WYWEB [23] | https://github.com/baudzhou/WYWEB | - |
| T5 | 句读<br>Punctuation | | | ✔ | | | Daizhige [33] | https://github.com/garychowcmu/daizhigev20 | - |
| T6 | 主题分类<br>Topic Classification | | | ✔ | | | Daizhige [33] | https://github.com/garychowcmu/daizhigev20 | - |
| T7 | 字词解释<br>Word Explanation | ✔ | | | | | Internet | - | - |
| T8 | 阅读理解<br>Reading Comprehension | | ✔ | | | | Internet | - | - |
| T9 | 虚词<br>Function Words | ✔ | | | | | Internet | - | - |
| T10 | 通假字<br>Homophones | | | | ✔ | | ACLUE [16] | https://github.com/isen-zhang/ACLUE | CC BY-NC-SA-4.0 |
| T11 | 单字多义<br>Polysemy | | | | ✔ | | ACLUE [16] | https://github.com/isen-zhang/ACLUE | CC BY-NC-SA-4.0 |
| T12 | 文言文写作<br>Ancient Prose Writing | | | | | ✔ | Model | - | - |
| T13-1 | 赏析真题<br>Appreciation Exam Questions | | ✔ | | | | Internet | http://ts300.5156edu.com/ | CC0 1.0 |
| T13-2 | 自由赏析<br>Free Appreciation | ✔ | | | | | Internet | https://www.gushixuexi.com/ | CC0 1.0 |
| T14-1 | 诗创作<br>Poetry Writing | | | | | ✔ | Model | - | - |
| T14-2 | 词创作<br>Ci Writing | | | | | ✔ | Model | - | - |
| T14-3 | 曲创作<br>Qu Writing | | | | | ✔ | Model | - | - |
| T15-1 | 内容问答<br>Content Q&A | ✔ | | | | | Internet | https://www.sou-yun.cn/ | CC0 1.0 |
| T15-2 | 题目作者问答<br>Title and Author Q&A | ✔ | | | | | Internet | https://www.sou-yun.cn/ | CC0 1.0 |
| T15-3 | 下句默写<br>Write the Next Sentence | ✔ | | | | | Internet | https://www.sou-yun.cn/ | CC0 1.0 |
| T15-4 | 上句默写<br>Write the Previous Sentence | ✔ | | | | | Internet | https://www.sou-yun.cn/ | CC0 1.0 |
| T15-5 | 理解性默写<br>Comprehension Dictation | | ✔ | | | | Internet | - | - |
| T15-6 | 判断体裁<br>Genre Judgment | ✔ | | | | | Internet | https://www.sou-yun.cn/ | CC0 1.0 |
| T16 | 古诗词翻译<br>Ancient Poetry Translation | ✔ | | | | | Internet | https://www.sou-yun.cn/ | CC0 1.0 |
| T17 | 情感分类<br>Sentiment Classification | | | | ✔ | | THU-FSPC [22] | https://github.com/THUNLP-AIPoet/Datasets | - |
| T18 | 古诗词英文翻译<br>Ancient Poetry to English | ✔ | | | | | Internet | https://www.sou-yun.cn/ | CC0 1.0 |
| T19 | 诗人介绍<br>Poet Introduction | ✔ | | | | | Internet | https://www.sou-yun.cn/ | CC0 1.0 |
| T20 | 意象解析<br>Analysis of Imagery | ✔ | | | | | Internet | https://www.sou-yun.cn/ | CC0 1.0 |
| T21-1 | 接下联<br>Couplet Following | ✔ | | | | | Internet | - | - |
| T21-2 | 对联创作<br>Couplet Writing | ✔ | | | | | Internet | - | - |
| T21-3 | 拟横批<br>HengPi Writing | ✔ | | | | | Internet | - | - |
| T22-1 | 近义词<br>Synonyms | ✔ | | | | | Internet | https://www.zdic.net/ | CC0 1.0 |
| T22-2 | 成语出处<br>The Origin of Idiom | ✔ | | | | | Internet | https://www.zdic.net/ | CC0 1.0 |
| T22-3 | 成语蕴含<br>Idiom Finding | ✔ | | | | | Internet | https://www.zdic.net/ | CC0 1.0 |
| T22-4 | 成语解释<br>Idiom Explanation | ✔ | | | | | Internet | https://www.zdic.net/ | CC0 1.0 |
| T23 | 谜语<br>Riddle | | ✔ | | | | Internet | - | - |
| T24 | 歇后语<br>Xiehouyu | ✔ | | | | | Internet | - | - |
| T25 | 古汉语音韵<br>Historical Chinese Phonology | | | | ✔ | | ACLUE [16] | https://github.com/isen-zhang/ACLUE | CC BY-NC-SA-4.0 |
| T26 | 国学常识问答<br>Knowledge of Sinology Q&A | ✔ | ✔ | | | | Internet | - | - |

## B.3 The Specific Source of the Data

Table 7 presents the data construction methods, data sources, links to the sources, and the licenses of the original data for 42 fine-grained tasks. Data without a specified source fall into three categories: (a) Questions and answers pairs manually constructed using knowledge content from the Internet; (b) Publicly available examination questions; (c) Questions and answers pairs generated by models.

In the data construction method column of Table 7, the following methods are described:

M1: Scraping and creating Q&A pairs from Internet sources; M2: Directly collecting relevant questions from various Internet platforms; M3: Manually constructing Q&A pairs based on existing CCLLA corpora; M4: Utilizing evaluation dataset questions from the CCLLA domain; M5: Using LLM-generated answers for assessment data.

For tasks without specified links, the sources fall into two categories: (a) Data generated by LLMs, providing responses based on training and inference capabilities. (b) Original data derived from widely accessible online knowledge texts, such as high school Chinese language resources, which are not restricted to specific webpages.

## B.4 The Construction of Dataset

### B.4.1 Specific Process

The process of constructing the dataset is divided into two stages: the data collection process and the data review process.

**(1) Data Collection Process**

**Data collected from the Internet.** Collectors: Volunteer A, Volunteer B, and Graduate Student C. Collection process: Based on common tasks in Chinese language exams (reading comprehension, comprehension dictation, *etc.*), collect high-quality exam questions made by experts and scholars from the Internet. We require that the questions should be closely matched with the tasks.

**Other open-source datasets.** Collector: Graduate Student D. Collection process: The data collector conducts research and collection of existing open-source datasets, selects classic tasks and high-quality data in the CCLLA field, and assesses the scarcity and construction difficulty of task data. Ultimately, valuable and currently scarce data are selected for reuse to supplement and improve the evaluation benchmark. Post-processing: (a) Text filtering: Perform operations such as handling of abnormal symbols (*e.g.*, blank squares), deletion of irrelevant content, and after manual inspection, obtain complete and high-quality data. (b) Question writing: Construct Q&A pairs oriented by the needs of different tasks. (c) Standardization of format: Questions for the same task are unified into the same questioning method. We directly set up corresponding question templates according to different tasks.

**LLMs.** Collector: Graduate Student E. Collection process: Tasks such as ancient poetry and prose writing are inherently open-ended and do not have fixed correct answers. For these tasks, we initially design a range of questions and then employ the ERNIE-3.5 [10] model to generate reference answers. These answers are further reviewed, filtered, and refined through a manual process.

**(2) Data Review Process**

a. Reviewers: The review is conducted by three individuals, namely Volunteer A, Graduate Student D, and Graduate Student E.

b. A total of approximately 7,000 data entries are collected. The reviewers manually verify the Q&A pairs from three dimensions: the standardization and accuracy of the questions, the correctness and comprehensiveness of the answers, and whether the text content contains ethical issues or unsafe content.

c. Data with non-standard Q&A are deleted or revised, and entries with safety issues are excluded.

d. Finally, the remaining data are rebalanced in terms of task and quality, followed by a second round of proofreading. In this second round, in addition to reviewing the safety and ethical aspects of the data, it is essential to ensure the accuracy and consistency of the Q&A content, with particular attention to grammar, punctuation, and adherence to proper expression standards. The data for the

26 tasks are also rebalanced to prevent excessive or insufficient data for certain tasks. For tasks with less data, additional backup data are incorporated to increase their volume, while for tasks with excessive data, low-quality entries are streamlined to maintain consistent overall quality. Each task's data undergoes at least two rounds of proofreading to ensure relevance, a balanced distribution of task difficulty, and ultimately, the comprehensiveness, accuracy, and safety of the dataset.

### B.4.2 Review Details

During the dataset construction process, two rounds of rigorous manual proofreading are conducted to ensure the accuracy and safety of all data. First, the initially collected 7,000 entries undergo the first round of review, where data containing non-standard Q&A pairs, inaccurate or incomplete answers, and entries with potential ethical issues or unsafe content are identified and filtered out. At this stage, 25.20% of the data are manually removed, leaving 5,236 entries. In the second round, the review standards are further refined, focusing not only on identifying additional safety and ethical concerns but also on ensuring the accuracy and consistency of the Q&A pairs, with special attention to the standardization of grammar, punctuation, and expression. An additional 6.89% of the data are removed during this phase, resulting in 4,875 entries that are comprehensive and secure. During this process, supplementary adjustments are made to address any data imbalances across individual tasks.

It is important to note that, despite our strict data review and filtering process, we cannot guarantee the complete absence of errors or safety issues in the dataset. Users should exercise caution and refrain from misusing the data. We emphasize the cultural value of this dataset and hope it provides valuable support and reference for research in the CCLLA field.

### B.4.3 Task Collection Requirements and Question Template Explanation

The following presents a simplified version of the task collection requirements and question template explanation adhered to by the collectors during the data collection process for 26 different tasks.

- **T1-Sentence Structure**

  **Collection Requirements:** Questions are constructed based on specific subcategories (inverted sentence structure, elliptical sentence, inverted sentence types, sentence structure identification). The questions should focus on identifying or analyzing particular features of the sentence structure, such as rearranging the order of inverted sentences, identifying omitted elements, or determining the sentence type. The questions must be clear and explicitly indicate the sentence features or components being examined.

  **Template Explanation:** The questions are in MCQ format. A classical Chinese sentence is provided, followed by a question that requires selecting the correct answer based on the features of the sentence.

- **T2-Classical Chinese to Modern Chinese**

  **Collection Requirements:** Select a sentence of classical Chinese and require its translation into modern Chinese. The question must clearly specify the classical Chinese text to be translated. The selected text should have a certain level of difficulty to assess translation skills and comprehension of the classical language.

  **Template Explanation:** The questions are in QA format. First, the translation task is presented, followed by the classical Chinese sentence that needs to be translated.

- **T3-Modern Chinese to Classical Chinese**

  **Collection Requirements:** Select sentences in modern Chinese and require their translation into classical Chinese. The question must clearly specify the modern Chinese text to be translated to assess the ability to convert it into classical Chinese.

  **Template Explanation:** The questions are in QA format. First, the translation task requirements are presented, followed by the modern Chinese text that needs to be converted.

- **T4-Named Entity Recognition**

  **Collection Requirements:** A sentence of classical Chinese is provided, and the task is to identify named entities within it. The entities include personal names, place names, book titles, or other proper nouns.

**Template Explanation:** The questions are in QA format. First, the entity recognition task is specified, followed by the classical Chinese text to be analyzed. The answer section lists all identified named entities.

- **T5-Punctuation**

   **Collection Requirements:** A sentence of classical Chinese that lacks punctuation is selected, and the task requires the subject to add appropriate punctuation marks. The question must clearly specify the text to be punctuated in order to assess understanding of the structure and content of the classical language.

   **Template Explanation:** The questions are in QA format. The prompt presents a punctuation-free classical Chinese sentence and requires the addition of the corresponding punctuation marks. The answer section provides the complete text with punctuation added.

- **T6-Topic Classification**

   **Collection Requirements:** A sentence of classical Chinese is provided, and the task requires classifying its content into the provided topic options. The question must include a clear list of topic classification options to assess the subject's understanding of the text's theme.

   **Template Explanation:** The questions are in QA format. The prompt includes a classical Chinese sentence without a theme label, and the subject is required to select the most relevant topic type from the given multiple-choice options. The answer section should contain the correct topic classification.

- **T7-Word Explanation**

   **Collection Requirements:** A word or phrase that requires explanation is selected from a classical Chinese sentence, and the task requires the subject to provide its modern Chinese explanation. The question must include the complete classical Chinese sentence to provide context for the word meaning explanation.

   **Template Explanation:** The questions are in QA format. First, a request for the explanation of the word or phrase is presented, followed by the classical Chinese sentence containing the word or phrase to be explained. Finally, the answer should provide the meaning of the selected word or phrase.

- **T8-Reading Comprehension**

   **Collection Requirements:** A passage of classical Chinese is selected, and relevant questions are designed to require the subject to respond based on the content of the text. The questions should focus on comprehension skills, such as extracting key information, analyzing character actions, or interpreting event outcomes.

   **Template Explanation:** The questions are in QA format. The prompt includes a passage of classical Chinese text along with a question related to its content, and the answer should provide a clear explanation or conclusion.

- **T9-Function Words**

   **Collection Requirements:** A sentence of classical Chinese containing key function words is selected, and the task requires the subject to determine the specific usage of the function word. The question must provide multiple options covering different usages of the function word to test understanding of its specific meaning in the context of classical Chinese.

   **Template Explanation:** The questions are in MCQ format. A classical Chinese sentence containing the target function word is presented, and the subject is required to choose the most appropriate usage from several explanation options. The answer section should include the correct option.

- **T10-Homophones**

   **Collection Requirements:** A sentence of classical Chinese containing homophones is selected, and the task requires the subject to clearly identify which character in the provided options is a homophone. The question must present multiple options, each containing a character that needs to be identified as a homophone.

   **Template Explanation:** The questions are in MCQ format. Several classical Chinese sentences containing homophones are provided, and the subject is required to choose the homophone from the options. The character to be identified is highlighted in the options, and the answer section includes the correct option.

- **T11-Polysemy**

**Collection Requirements:** A Chinese character with multiple meanings is selected, and questions are constructed based on the various interpretations of this character in different sentences. The questions may require selecting the explanation that corresponds to a given meaning from a specific sentence or identifying the usage that expresses a particular meaning among multiple sentences. This type of question assesses students' ability to understand the context of polysemous words.

**Template Explanation:** The questions are in MCQ format. The first format presents a specific sentence containing the polysemous character and requires the subject to choose the option that best matches a specific explanation of the character in that sentence. The options list the different meanings of the polysemous character. The second format provides multiple sentences with the polysemous character highlighted and requires the subject to identify the sentence that corresponds to the meaning described in the question. Each option demonstrates the usage of the polysemous character in different contexts.

- **T12-Ancient Prose Writing**

  **Collection Requirements:** Tasks should be designed to involve writing in classical Chinese across various scenarios or themes, requiring the subject to compose in classical Chinese. These scenarios can include storytelling, writing social media copy, or marketing content. The aim is to assess the subject's ability to create prose in classical Chinese and their skill in using ancient language to express modern ideas flexibly.

  **Template Explanation:** The questions are in QA format. The prompt is typically presented in the form of a first-person request or command, asking for the creation of a specific type of content, such as a story or copywriting, with the requirement to use classical Chinese. The question structure includes a detailed scenario setup and specific writing instructions, and the answer usually provides a complete prose text in classical Chinese.

- **T13-Appreciation**

  **Collection Requirements:** Questions should focus on the interpretation and analysis of ancient Chinese poetry. The questions may ask participants to identify incorrect interpretations or analyze imagery, metaphors, and the emotions conveyed in the poem. Through these tasks, subjects need to demonstrate a deep understanding of the language, structure, and cultural background of the poetry.

  **Template Explanation:** The questions are in QA and MCQ formats. In the first format (MCQ), the question provides a classical poem and annotations, with the options presenting different interpretations or analyses. The subject must select the best answer. In the second format (QA), the question provides a poem along with an open-ended question, asking the subject to analyze the metaphors, imagery, and emotions expressed in the poem. The answer should include an analysis from one or more perspectives, explaining the deeper meanings and emotions of the poem.

- **T14-Ancient Poetry Writing**

  **Collection Requirements:** Tasks should involve the creation of ancient Chinese poetry or lyrical works. The question must provide a clear theme and specify the required form, such as the genre of the poem, the name of the tune, or the title of the melody. Participants are required to compose within the traditional metrical structure, assessing their language expression skills and mastery of ancient poetry forms.

  **Template Explanation:** The questions are in QA format. The prompt usually takes the form of a direct request, requiring the creation of a poem, lyric, or song based on a specified theme and format. The question description includes the theme and specific creative form, while the answer should provide a complete example that meets the task requirements.

- **T15-Basic Q&A**

  **Collection Requirements:** Tasks should cover fundamental questions on various aspects of ancient Chinese poetry. These may include questions on full-text content, identifying the poet and title, completing the next or previous line, understanding-based completion, and determining the genre. The questions must clearly specify the requirements, demonstrating the participant's knowledge of specific aspects of classical poetry.

  **Template Explanation:** (a) Content Q&A: The questions are in QA format. The question directly asks for the full content of a given poem and requires participants to write it out completely. (b) Title and Author Q&A: The questions are in QA format. The task presents a part of a classical poem and asks the participant to identify its title and author. (c) Write the Next Sentence: The questions are in QA format. The question provides the first line and asks the participant to write the next line of the poem. (d) Write the Previous Sentence: The questions are in QA format. The question gives

the second line and asks the participant to fill in the previous one. (e) Comprehension Dictation: The questions are in FB format. The task provides a hint, such as the poem's theme or function, and asks for a key line from the poem to be filled in. (f) Genre Judgment: The questions are in QA format. The question presents a poem and requires the participant to determine its genre.

- **T16-Ancient Poetry Translation**

  **Collection Requirements:** The task should involve translating ancient Chinese poetry into modern vernacular Chinese. The question aims to assess the understanding of ancient poetic language and the ability to accurately convert it into more accessible modern language, ensuring the translation conveys the original imagery and emotions.

  **Template Explanation:** The questions are in QA format. The question asks subject to translate the given classical poem into vernacular Chinese. It includes the classical poem text, and the answer provides the modern vernacular translation of the poem.

- **T17-Sentiment Classification**

  **Collection Requirements:** The task should assess the ability to understand and classify the sentiment of ancient Chinese poetry. Each question provides a poem and asks the subject to choose the most appropriate sentiment classification from the given options. These options typically include negative, slightly negative, neutral, slightly positive, and positive.

  **Template Explanation:** The questions are in MCQ format. The question provides the title, author, and content of the poem, then requires the subject to select the sentiment classification that best matches the emotional tone. The options range from "negative" to "positive", and the answer identifies the best choice.

- **T18-Ancient poetry to English**

  **Collection Requirements:** The task is designed to test the ability to translate ancient Chinese poetry into English. Each question should provide a specific ancient poem and require the subject to translate it into modern English, ensuring that the translation preserves the original imagery and emotional expression of the poem.

  **Template Explanation:** The questions are in QA format. The task presents an ancient poem and asks for its translation into English. The question section includes the given poem, while the answer section provides a complete English translation.

- **T19-Poet introduction**

  **Collection Requirements:** The task is designed to guide the introduction of the biographies, achievements, and influences of ancient poets. Each question should present a clear request for detailed information about the specified poet, including their background, representative works, and literary contributions.

  **Template Explanation:** The questions are in QA format. The question asks for the background and achievements of a particular poet. The question section consists of a simple request for an introduction, with the specific poet designated by the creator. The answer provides a detailed biography of the poet, including birth and death years, major experiences, literary achievements, and their historical and cultural significance.

- **T20-Analysis of imagery**

  **Collection Requirements:** The task is designed to analyze the multiple meanings of common imagery found in ancient poetry or prose. The questions should highlight specific imagery and require an explanation of its traditional and symbolic significance in different historical texts, thereby assessing the depth of understanding of the imagery.

  **Template Explanation:** The questions are in QA format. The question asks for a list of the multiple meanings typically associated with the imagery. The question section includes the name of the imagery, while the answer section provides a detailed explanation of the imagery within the context of ancient texts, potentially listing various meanings along with historical background and classic citation examples.

- **T21-Couplet**

  **Collection Requirements:** The task is designed to test or inspire the understanding and skills in couplet following, couplet writing, or HengPi writing. The questions may involve providing the lower line for a given upper line, composing a couplet based on a festival or theme, or creating a suitable HengPi for an existing couplet. The task assesses mastery of parallel structure and creative interpretation of the couplet's theme.

**Template Explanation:** The questions are in QA format. The task may ask for couplet following by matching a lower line to an upper line, couplet writing based on a specific festival or theme, or HengPi writing for a given couplet. The question provides the necessary information and challenge, while the answer presents a couplet, matching lower line, or HengPi that meets the specified requirements.

- **T22-Idiom**

   **Collection Requirements:** The task is designed to assess understanding, explanation, origin, and association of idioms. These questions may involve identifying synonyms of idioms, providing an idiom explanation, finding the origin of the idiom, or locating idioms embedded in a text and explaining their meaning. This task demonstrates the subject's familiarity with idioms and their ability to apply them.

   **Template Explanation:** The questions are in QA format. The question format involves identifying synonyms of a given idiom, providing an idiom explanation, stating the origin of idiom, or finding idioms in a text and explaining them. Each question provides an idiom or text, and the answer includes relevant idiom information such as synonyms, explanations, origin, or the idioms embedded in the text and their meanings.

- **T23-Riddle**

   **Collection Requirements:** The task is designed to present engaging and challenging riddles, covering a variety of subjects or themes. The goal is to test the subject's associative thinking, breadth of knowledge, and problem-solving skills. Each riddle should provide sufficient clues for logical reasoning to arrive at the correct answer.

   **Template Explanation:** The questions are in QA format. The subject is typically asked to guess an ancient historical figure, place name, country, or other objects. The question consists of the riddle and specific requirements, while the answer section provides the correct solution.

- **T24-Xiehouyu**

   **Collection Requirements:** The task is designed to assess the understanding and recall of Xiehouyu. The question usually provides the first part of the Xiehouyu, and the subject is required to complete the second part. This type of question evaluates familiarity with traditional Chinese cultural expressions of humor and wisdom, as well as the ability to interpret metaphors.

   **Template Explanation:** The questions are in QA format. The question is presented as an incomplete Xiehouyu, usually giving only the first part, and the subject is asked to complete the saying. The question clearly specifies the need for completion, and the answer section provides the second part of the Xiehouyu, optionally with an explanation.

- **T25-Historical Chinese phonology**

   **Collection Requirements:** The task involves designing multiple-choice questions on the topic of historical Chinese phonology. These questions require the subject to have a basic understanding of ancient Chinese phonetic structures, including initials and finals, as well as the Qieyun system. The subject must be able to accurately recognize and evaluate phonological phenomena and classical works in ancient Chinese phonology.

   **Template Explanation:** The questions are in MCQ format. Each question presents a specific knowledge point related to ancient Chinese phonology, along with several options, requiring the subject to select the correct answer. Each question includes a prompt and several answer choices, and the answer section identifies the correct option.

- **T26-Knowledge of Sinology Q&A**

   **Collection Requirements:** The task involves designing questions covering a wide range of common knowledge in Sinology, including history, geography, linguistics, philosophy, and other aspects of ancient Chinese culture. The questions should be concise and clearly formulated to assess the subject's understanding of traditional Chinese culture.

   **Template Explanation:** The questions are in QA and MCQ formats. Each question should clearly present a specific topic related to Sinology in a straightforward manner. The question section contains a clear prompt, while the answer section provides a concise and accurate response.

# C Evaluation

## C.1 Evaluated Model

Details of all evaluated models are shown in Table 8.

Table 8: Details of all evaluated models. Zoom in for better view.

| Model | #Parameter | Base Model | Creator | Access | Website | Domain |
|---|---|---|---|---|---|---|
| Baichuan2-7B-Chat [42] | 7B | - | Baichuan | Weights | https://huggingface.co/baichuan-inc/Baichuan2-7B-Chat | General |
| Baichuan2-13B-Chat [42] | 13B | - | Baichuan | Weights | https://huggingface.co/baichuan-inc/Baichuan2-13B-Chat | General |
| Firefly-Baichuan2-13B [54] | 13B | Baichuan2-13B-Base [42] | YeungNLP | Weights | https://huggingface.co/YeungNLP/firefly-baichuan2-13b | General |
| InternLM2-Chat-7B [55] | 7B | - | Internlm | Weights | https://huggingface.co/internlm/internlm2-chat-7b | General |
| Qwen1.5-0.5B-Chat [41] | 0.5B | - | Alibaba | Weights | https://huggingface.co/Qwen/Qwen1.5-0.5B-Chat | General |
| Qwen1.5-4B-Chat [41] | 4B | - | Alibaba | Weights | https://huggingface.co/Qwen/Qwen1.5-4B-Chat | General |
| Qwen1.5-7B-Chat [41] | 7B | - | Alibaba | Weights | https://huggingface.co/Qwen/Qwen1.5-7B-Chat | General |
| Qwen1.5-14B-Chat [41] | 14B | - | Alibaba | Weights | https://huggingface.co/Qwen/Qwen1.5-14B-Chat | General |
| Qwen1.5-32B-Chat [41] | 32B | - | Alibaba | Weights | https://huggingface.co/Qwen/Qwen1.5-32B-Chat | General |
| Qwen1.5-72B-Chat [41] | 72B | - | Alibaba | Weights | https://huggingface.co/Qwen/Qwen1.5-72B-Chat | General |
| Yi-1.5-6B-Chat [52] | 6B | - | 01.AI | Weights | https://huggingface.co/01-ai/Yi-1.5-6B-Chat | General |
| Yi-1.5-9B-Chat [52] | 9B | - | 01.AI | Weights | https://huggingface.co/01-ai/Yi-1.5-9B-Chat | General |
| Yi-1.5-34B-Chat [52] | 34B | - | 01.AI | Weights | https://huggingface.co/01-ai/Yi-1.5-34B-Chat | General |
| ChatGLM2-6B [43] | 6B | - | Tsinghua | Weights | https://huggingface.co/THUDM/chatglm2-6b | General |
| ChatGLM3-6B [43] | 6B | - | Tsinghua | Weights | https://huggingface.co/THUDM/chatglm3-6b | General |
| Ziya-LLaMA-13B-v1.1 [58] | 13B | Llama-13B [39] | IDEA | Weights | https://huggingface.co/IDEA-CCNL/Ziya-LLaMA-13B-v1.1 | General |
| LLaMA2-7B-Chat [40] | 7B | - | Meta | Weights | https://huggingface.co/meta-llama/Llama-2-7b-chat-hf | General |
| LLaMA2-13B-Chat [40] | 13B | - | Meta | Weights | https://huggingface.co/meta-llama/Llama-2-13b-chat-hf | General |
| LLaMA2-Chinese-7B-Chat [45] | 7B | LLaMA2-7B [40] | FlagAlpha | Weights | https://huggingface.co/FlagAlpha/Llama2-Chinese-7b-Chat | General |
| LLaMA2-Chinese-13B-Chat [46] | 13B | LLaMA2-13B [40] | FlagAlpha | Weights | https://huggingface.co/FlagAlpha/Llama2-Chinese-13b-Chat | General |
| LLaMA3-8B-Instruct [59] | 8B | - | Meta | Weights | https://huggingface.co/meta-llama/Meta-Llama-3-8B-Instruct | General |
| LLaMA3-Chinese-8B-Chat [60] | 8B | LLaMA3-8B [59] | Shenzhi-wang | Weights | https://huggingface.co/shenzhi-wang/Llama3-8B-Chinese-Chat | General |
| Gemma-1.1-7B-IT [57] | 7B | - | Google | Weights | https://huggingface.co/google/gemma-1.1-7b-it | General |
| Bloom-7B-Chunhua [48] | 7B | Bloom-7B1 [65] | - | Weights | https://modelscope.cn/models/wptoux/bloom-7b-chunhua | Ancient Chinese |
| Ancient-Chat-LLM-7B [51] | 7B | Internlm-7B [55] | - | Weights | https://modelscope.cn/models/HinGwenWoong/ancient-chat-7b | Ancient Chinese |
| Xunzi-Qwen1.5-7B [47] | 7B | Qwen-1.5-7B [41] | Nanjing Agricultural University | Weights | https://modelscope.cn/models/Xunzillm4cc/Xunzi-Qwen1.5-7B_chat | Ancient Chinese |
| ERNIE-3.5-8K-0329 [10] | - | - | Baidu | API | https://console.bce.baidu.com/qianfan | General |
| ERNIE-4.0-8K-0329 [10] | - | - | Baidu | API | https://console.bce.baidu.com/qianfan | General |
| Spark-3.5 [56] | - | - | iFLYTEK | API | https://xinghuo.xfyun.cn/sparkapi | General |
| GPT-3.5 [61] | - | - | OpenAI | API | https://chat.openai.com/ | General |
| GPT-4 [62] | - | - | OpenAI | API | https://chat.openai.com/ | General |

## C.2 Scoring Prompt

Scoring prompts of various tasks are illustrated in Figure 13.

## C.3 Scoring Consistency Analysis

When evaluating models using scoring methods, it is common to conduct an analysis of the consistency between model scores and human assessments [66, 67, 68, 69]. For the model scoring results on the WenMind benchmark, we have chosen to use the "Agreement Rate" metric to determine whether the model scores align with human expectations.

**Agreement Rate.** We present the question, reference answer, model output, model score, and the reasoning behind the model's score to humans. The humans then decide whether they agree based on the "score" and the "reasoning". If the model's score is reasonable and aligns with human understanding and expectations, they indicate "Agreement". If there are scoring errors or contradictory reasons, they indicate "Disagreement". The overall consistency between the model's scores and human expectations is determined by the proportion of "Agreements". This method, compared to having humans score or rank directly for comparison, more directly reflects consistency, reduces the influence of human subjectivity, speeds up the annotation process, and minimizes errors due to the difficulty of human ranking or scoring.

We randomly sample 417 data points (stratified by tasks) from the scoring results of five representative LLMs, and three volunteers perform "Consistency Judgments" on these samples. We then average the "Agreement Rate" of all volunteers. According to Table 9, the average agreement rate across the five LLMs is **89.4%**. This indicates that our model scoring method has a high level of consistency with human expectations, providing results that are of significant reference value. In the end, we opts for ERNIE-3.5 [10] as the scoring model, with the cost of completing one round of scoring being approximately $3.8, striking a balance between cost and effectiveness.

 学生作答内容：A、动词，做  参考答案：D  根据参考答案，如果学生作答内容正确，得分为1，否则得分为0。
输出格式要求如下，不要输出其他内容：["得分"] 例如：["0"]或["1"]

Student's answer content: A、动词，做  Reference answer: D
According to the reference answer, if the student's answer is correct, the score is 1, otherwise the score is 0.
Output format is required as follows, do not output other content: ["Score"]  For example: ["0"] or ["1"]

 ["0"]

---

 学生作答内容：AC  参考答案：A、C
根据参考答案：（1）学生作答内容[全部正确]，得分为1；（2）如果学生作答内容[只包含一个选项且选项正确]，得分为0.5；（3）学生作答内容中[包含错误选项]，得分为0。 输出格式要求如下，不要输出其他内容，严格按照输出格式要求：["得分"] 例如：["1"]或["0.5"]或["0"]

Student's answer content: AC  Reference answer: A, C
According to the reference answer: (1) If the student's answer content is [completely correct], the score is 1; (2) If the student's answer content [only includes one option and the option is correct], the score is 0.5; (3) If the student's answer content [includes incorrect options], the score is 0. The output format is required as follows, do not output other content, strictly follow the output format requirements: ["Score"] For example: ["1"] or ["0.5"] or ["0"]

 ["1"]

---

 学生作答内容："微斯人，吾谁与归？" 的正确语序应该是 "吾谁与归？微斯人。" 参考答案：微斯人，吾与谁归
问题：微斯人，吾谁与归？上面倒装句的正确语序是什么？
你是严谨、遵守规则的改卷老师。请根据上面的问题、参考答案和学生作答内容，完成以下任务。
（1）将参考答案分解为若干个得分点。（2）根据每个得分点，判断学生作答内容中是否有涉及得分点的内容。（3）给出参考答案中你设置的得分点个数，学生作答内容中符合得分点的个数，以及相关理由。（4）学生作答内容中符合得分点的个数要≤参考答案中你设置的得分点个数（至少为1）。
输出格式要求如下，请不要输出其他额外内容，严格按照输出格式要求：
["参考答案中你设置的得分点个数","学生作答内容中符合得分点的个数","理由"] 例如：["4","3","理由…"]

Student's answer content: "微斯人，吾谁与归？" The correct word order should be "吾谁与归？微斯人。" Reference answer: 微斯人，吾与谁归 Question: What is the correct word order for the inverted sentence "微斯人，吾谁与归？"? You are a rigorous and rule-abiding grading teacher. Please complete the following tasks based on the above question, reference answer, and student's answer content. (1) Break down the reference answer into several scoring points. (2) Based on each scoring point, determine whether the student's answer content involves the content of the scoring point. (3) Provide the number of scoring points you set in the reference answer, the number of points in the student's answer content that meet the scoring points, and the relevant reasons. (4) The number of points in the student's answer content that meet the scoring points should be ≤ the number of scoring points you set in the reference answer (at least 1). The output format is required as follows, please do not output other additional content, strictly follow the output format requirements: ["Number of scoring points you set in the reference answer", "Number of points in the student's answer content that meet the scoring points", "Reasons"] For example: ["4", "3", "Reasons…"]

 ["1","0","参考答案的得分点在于学生能否正确地将倒装句"微斯人，吾谁与归？"还原为正常语序。正确答案应为"微斯人，吾与谁归"。学生作答内容虽然尝试调整语序，但并未给出正确答案，而是错误地调整为"吾谁与归？微斯人。"，这既不符合语法规则，也没有正确还原句子的原意。因此，学生作答内容中并没有涉及任何得分点。"]

["1","0","The scoring point in the reference answer lies in whether the student can correctly restore the inverted sentence '微斯人，吾谁与归？' to its normal word order. The correct answer should be '微斯人，吾与谁归'. Although the student's answer attempted to adjust the word order, it did not provide the correct answer but incorrectly adjusted it to '吾谁与归？微斯人。', which does not conform to grammatical rules and does not correctly restore the original meaning of the sentence. Therefore, there are no scoring points involved in the student's answer content."]

---

 学生作答内容：悔恨，懊悔，反省，悔改  参考答案：悔过自责（答案不唯一，符合题目要求即可，内容仅供参考） 问题：给出下面成语的近义词：悔过自忏 要求：学生作答内容必须为成语，只要作答的"成语"与问题中的"成语"为近义词就判为正确。 请根据创作要求（参考答案仅供打分参考）对学生作答内容进行打分，分数范围从0到1。 输出格式要求如下，不要输出其他内容，严格按照输出格式要求：["得分","理由"] 例如：["0.8","理由…"]

Student's answer content: Remorse, Regret, Introspection, Repentance Reference answer: Self-reproach for past errors (The answer is not unique, as long as it meets the requirements of the question, the content is for reference only) Question: Provide a near-synonym for the idiom "悔过自责" Requirement: The student's answer must be an idiom, and as long as the "idiom" in the answer is a near-synonym to the "idiom" in the question, it will be judged as correct. Please score the student's answer content according to the creative requirements (the reference answer is only for scoring reference), with a score ranging from 0 to 1. The output format is required as follows, do not output other content, strictly follow the output format requirements: ["Score", "Reason"] For example: ["0.8", "Reason…"]

 ["0", "理由：学生作答的内容"悔恨，懊悔，反省，悔改"虽然都与悔过自忏有一定的关联，但它们并非成语，而是由多个词汇组成的短语或表达。根据题目要求，学生作答的内容必须为成语，且该成语需要与问题中的成语"悔过自忏"为近义词。因此，学生的作答内容不符合题目要求，未能提供成语作为近义词，所以得分为0。"]

["0", "Reason: Although the student's answers 'regret, remorse, introspection, repentance' are all related to repentance, they are not idioms but phrases or expressions composed of multiple words. According to the requirements of the question, the student's answer must be an idiom, and this idiom needs to be a synonym for the idiom 'repentance' in the question. Therefore, the student's answer does not meet the requirements of the question, did not provide an idiom as a synonym, so the score is 0."]

---

 学生作答内容：人民共日月齐光。  参考答案：江山共日月争辉。（答案不唯一，符合题目要求即可，内容仅供参考） 问题：根据上联对出下联：祖国与天地同寿 创作要求：从五个方面进行评判，1、字数是否与上联相同，2、相同位置是否词性相同，3、是否平仄相合，4、与上联是否过渡自然，5、是否表述优美音韵和谐。 请根据创作要求（参考答案仅供打分参考）对学生作答内容进行打分，分数范围从0到1。
输出格式要求如下，不要输出其他内容，严格按照输出格式要求：["得分", "理由"] 例如：["0.8","理由…"]

["0.5", "Reason: The student's answer 'People share the light with the sun and moon' does not meet the requirements of the couplet. The reference answer 'Rivers and mountains compete with the sun and moon for brilliance' is given as a guideline. The evaluation criteria include: 1) whether the number of characters matches the upper couplet, 2) whether the parts of speech are the same at corresponding positions, 3) whether the tones are harmonious, 4) whether the transition is natural from the upper couplet, and 5) whether the expression is elegant and the rhyme is harmonious. The student's answer partially meets the criteria, with the number of characters and the transition being natural, but it does not achieve harmony in tone or elegance in expression, thus the score is 0.5."]

 ["0.4", "理由：学生作答的"人民共日月齐光"与上联"祖国与天地同寿"在字数上相符，满足了第一个评判标准。但在词性对应上，"人民"与"祖国"都是名词，但"天地"作为一组并列概念，与"日月"这一组并列概念的对应关系并不完全吻合，因此第二个标准未达到完全匹配。在平仄方面，学生的下联并未完全遵循平仄相合的原则，尤其是尾字"光"与上联的"寿"在平仄上并未形成对立或和谐的关系，因此第三个标准未能满足。在过渡自然和表述优美音韵和谐方面，下联的"人民共日月齐光"虽然表达了人民与日月同辉的美好愿景，但与上联"祖国与天地同寿"的过渡并不十分自然，且在音韵和谐上也略显生硬。综上，学生的下联在字数上相符，但在词性对应、平仄相合、过渡自然和音韵和谐方面存在一定不足。"]

["0.4", "Reason: The student's response 'People share the light with the sun and moon' matches the number of characters with the upper couplet 'The motherland shares longevity with heaven and earth', fulfilling the first criterion. However, in terms of part-of-speech correspondence, while 'people' and 'motherland' are both nouns, the corresponding relationship between 'heaven and earth' as another pair of concepts and 'sun and moon' is not a complete match, thus the second criterion is not fully met. Regarding the principle of level and oblique tones, the student's lower couplet does not fully adhere to the principle of tonal harmony, especially with the final character 'light' not forming an opposing or harmonious relationship with 'longevity' on the upper couplet, so the third criterion is not met. In terms of natural transition and elegant expression with harmonious rhyme, although the lower couplet 'People share the light with the sun and moon' conveys a beautiful vision of people shining alongside the sun and moon, it does not transition very naturally from the upper couplet 'The motherland shares longevity with heaven and earth', and the rhyme also appears somewhat stiff. In summary, the student's lower couplet matches the character count but falls short in terms of part-of-speech correspondence, tonal harmony, natural transition, and harmonious rhyme."]

Figure 13: Scoring prompt samples.

Table 9: The rate of human agreement with the model scoring.

| Model | Overall | Domain | | | Capability | | |
|---|---|---|---|---|---|---|---|
| | | Ancient Prose | Ancient Poetry | Ancient Literary Culture | Understanding | Generation | Knowledge |
| Baichuan2-7B-Chat [42] | 0.890 | 0.905 | 0.886 | 0.873 | 0.920 | 0.700 | 0.938 |
| GPT-3.5 [61] | 0.909 | 0.912 | 0.913 | 0.900 | 0.909 | 0.857 | 0.931 |
| LLaMA3-Chinese-8B-Chat [60] | 0.882 | 0.891 | 0.894 | 0.855 | 0.893 | 0.743 | 0.931 |
| Qwen1.5-7B-Chat [41] | 0.882 | 0.912 | 0.881 | 0.845 | 0.914 | 0.686 | 0.931 |
| Spark-3.5 [56] | 0.906 | 0.878 | 0.881 | 0.982 | 0.898 | 0.786 | 0.969 |
| Average | 0.894 | 0.899 | 0.891 | 0.891 | 0.907 | 0.754 | 0.940 |

Table 10: Traditional metrics on translation and punctuation tasks. See Appendix B.1 for details on the tasks represented by T2, T3, T5, T16, and T18. Zoom in for better view.

| Model | T2 | | | | T3 | | | | T16 | | | | T18 | | | | T5 |
|---|---|---|---|---|---|---|---|---|---|---|---|---|---|---|---|---|---|
| | ROUGE-1 | ROUGE-2 | ROUGE-L | BLEU | ROUGE-1 | ROUGE-2 | ROUGE-L | BLEU | ROUGE-1 | ROUGE-2 | ROUGE-L | BLEU | ROUGE-1 | ROUGE-2 | ROUGE-L | BLEU | F1 |
| Baichuan2-7B-Chat [42] | 56.5 | 28.9 | 53.4 | 17.8 | 53.4 | 23.9 | 51.3 | 11.7 | 48.3 | 17.5 | 40.0 | 9.2 | 78.3 | 57.5 | 73.7 | 36.9 | 62.0 |
| Baichuan2-13B-Chat [42] | 53.0 | 24.3 | 49.8 | 13.3 | 49.2 | 19.6 | 47.5 | 7.8 | 47.1 | 16.5 | 38.4 | 8.8 | 78.5 | 57.9 | 73.8 | 38.2 | 59.8 |
| Firefly-Baichuan2-13B [54] | 54.2 | 28.7 | 51.6 | 17.5 | 35.8 | 16.2 | 34.3 | 5.9 | 52.0 | 21.5 | 44.9 | 11.6 | 79.2 | 57.6 | 73.5 | 36.8 | 49.7 |
| ChatGLM2-6B [43] | 44.6 | 19.3 | 40.9 | 8.8 | 41.2 | 14.7 | 39.1 | 4.6 | 42.6 | 11.7 | 32.5 | 4.1 | 65.1 | 48.5 | 59.6 | 27.0 | 37.2 |
| ChatGLM3-6B [43] | 52.6 | 24.5 | 49.0 | 13.1 | 53.3 | 23.5 | 51.0 | 9.6 | 47.1 | 15.4 | 37.6 | 7.1 | 72.3 | 53.6 | 67.9 | 31.6 | 61.9 |
| InternLM2-Chat-7B [55] | 60.3 | 33.0 | 57.6 | 21.1 | 64.2 | 36.4 | 61.9 | 22.1 | 49.5 | 17.6 | 41.1 | 8.8 | 77.1 | 56.7 | 72.2 | 35.7 | 63.0 |
| Qwen1.5-05B-Chat [41] | 56.3 | 27.5 | 53.4 | 14.5 | 54.4 | 24.6 | 52.5 | 11.1 | 50.6 | 19.7 | 46.9 | 8.8 | 69.7 | 47.6 | 65.1 | 25.8 | 30.8 |
| Qwen1.5-4B-Chat [41] | 60.0 | 32.1 | 57.2 | 19.7 | 62.3 | 33.9 | 60.0 | 19.4 | 53.9 | 21.3 | 46.4 | 11.8 | 75.8 | 54.5 | 71.3 | 33.9 | 59.6 |
| Qwen1.5-7B-Chat [41] | 54.5 | 27.5 | 50.8 | 25.4 | 56.6 | 27.8 | 53.8 | 29.1 | 46.8 | 15.3 | 37.4 | 10.7 | 73.8 | 55.1 | 68.3 | 27.0 | 55.4 |
| Qwen1.5-14B-Chat [41] | 55.4 | 28.0 | 52.0 | 15.9 | 57.7 | 29.2 | 55.0 | 17.0 | 45.5 | 14.6 | 35.7 | 6.6 | 72.6 | 54.0 | 67.9 | 33.7 | 61.9 |
| Qwen1.5-32B-Chat [41] | 61.0 | 33.1 | 57.8 | 21.4 | 63.7 | 36.6 | 61.3 | 22.7 | 50.7 | 18.3 | 42.2 | 10.1 | 75.7 | 55.4 | 70.5 | 35.3 | 64.9 |
| Qwen1.5-72B-Chat [41] | 58.2 | 30.2 | 54.6 | 17.9 | 61.2 | 32.8 | 58.5 | 19.4 | 47.1 | 15.9 | 37.4 | 7.5 | 78.0 | 57.8 | 73.5 | 37.3 | 61.9 |
| Yi-1.5-6B-Chat [52] | 51.4 | 24.7 | 47.4 | 13.7 | 50.6 | 21.5 | 48.0 | 8.6 | 49.4 | 17.2 | 40.3 | 8.4 | 76.1 | 55.0 | 70.8 | 32.9 | 50.2 |
| Yi-1.5-9B-Chat [52] | 53.4 | 26.7 | 50.1 | 15.7 | 57.7 | 28.3 | 55.2 | 13.8 | 48.1 | 17.1 | 39.3 | 8.4 | 75.9 | 53.7 | 69.8 | 32.1 | 60.1 |
| Yi-1.5-34B-Chat [52] | 54.1 | 28.3 | 50.6 | 16.1 | 55.9 | 27.4 | 53.5 | 13.1 | 46.6 | 16.0 | 37.3 | 7.0 | 78.0 | 57.3 | 72.3 | 35.8 | 60.7 |
| ERNIE-3.5-8K-0329 [10] | 53.0 | 26.1 | 49.1 | 15.2 | 54.3 | 30.0 | 51.5 | 19.0 | 36.9 | 9.1 | 26.2 | 1.4 | 76.5 | 55.8 | 70.5 | 35.6 | 72.4 |
| ERNIE-4.0-8K-0329 [10] | 58.7 | 31.3 | 55.6 | 19.8 | 60.1 | 38.8 | 57.7 | 28.0 | 45.6 | 13.7 | 35.5 | 5.6 | 70.1 | 52.9 | 65.6 | 34.1 | 61.2 |
| Spark-3.5 [56] | 61.5 | 35.2 | 58.4 | 24.4 | 58.8 | 35.7 | 56.2 | 25.1 | 48.3 | 16.3 | 39.3 | 7.6 | 49.3 | 35.5 | 45.3 | 21.0 | 76.2 |
| Gemma-11-7b-IT [57] | 32.4 | 10.3 | 28.7 | 2.0 | 32.8 | 10.3 | 30.6 | 1.9 | 44.7 | 12.9 | 35.7 | 4.2 | 75.3 | 54.3 | 69.9 | 30.0 | 13.4 |
| Ziya-LLaMA-13B-v1.1 [58] | 63.1 | 35.5 | 60.4 | 23.4 | 39.4 | 18.6 | 38.2 | 9.0 | 53.3 | 20.9 | 46.5 | 11.4 | 77.4 | 55.7 | 73.3 | 34.0 | 57.4 |
| LLaMA2-7B-Chat [40] | 10.1 | 1.3 | 9.8 | 0.0 | 15.0 | 2.5 | 14.3 | 0.0 | 3.2 | 0.4 | 3.0 | 0.0 | 72.4 | 51.5 | 66.7 | 13.2 | 0.1 |
| LLaMA2-13B-Chat [40] | 10.4 | 1.3 | 10.1 | 0.0 | 14.9 | 2.7 | 14.3 | 0.0 | 17.9 | 2.2 | 16.8 | 0.0 | 65.3 | 49.8 | 59.9 | 20.0 | 1.6 |
| LLaMA2-Chinese-7B-Chat [45] | 19.6 | 6.9 | 17.6 | 2.0 | 44.5 | 19.1 | 42.9 | 6.0 | 10.3 | 3.0 | 9.2 | 0.6 | 74.2 | 52.1 | 67.5 | 27.8 | 20.2 |
| LLaMA2-Chinese-13B-Chat [46] | 43.8 | 16.8 | 40.0 | 7.8 | 52.2 | 23.0 | 49.9 | 10.0 | 44.1 | 13.3 | 34.6 | 5.1 | 72.1 | 51.3 | 67.4 | 30.9 | 51.4 |
| LLaMA3-8B-Instruct [59] | 15.3 | 3.0 | 14.5 | 0.3 | 28.9 | 6.3 | 27.4 | 0.0 | 18.9 | 3.6 | 17.0 | 0.2 | 60.0 | 49.4 | 56.2 | 14.3 | 11.1 |
| LLaMA3-Chinese-8B-Chat [60] | 50.9 | 23.2 | 47.5 | 12.2 | 49.5 | 20.5 | 47.6 | 8.4 | 47.8 | 15.4 | 38.6 | 7.3 | 71.9 | 55.7 | 67.3 | 35.2 | 57.7 |
| GPT-3.5 [61] | 51.2 | 22.8 | 47.5 | 12.3 | 48.1 | 18.0 | 45.4 | 7.2 | 48.7 | 16.6 | 40.1 | 8.5 | 78.6 | 57.7 | 73.9 | 38.2 | 58.0 |
| GPT-4 [62] | 46.9 | 20.4 | 43.2 | 9.6 | 50.6 | 21.1 | 48.6 | 9.5 | 42.8 | 12.2 | 32.5 | 4.0 | 78.4 | 56.8 | 71.8 | 35.8 | 62.3 |
| Ancient-Chat-LLM-7B [51] | 58.8 | 31.2 | 56.3 | 19.3 | 61.3 | 34.8 | 59.4 | 22.0 | 48.8 | 16.5 | 41.8 | 7.6 | 73.4 | 46.8 | 68.8 | 22.3 | 67.3 |
| Bloom-7B-Chunhua [48] | 55.4 | 26.4 | 52.4 | 14.5 | 56.0 | 27.3 | 53.6 | 11.8 | 49.7 | 18.1 | 42.4 | 8.8 | 75.1 | 54.7 | 71.3 | 32.8 | 62.0 |
| Xunzi-Qwen1.5-7B [47] | 65.9 | 38.5 | 63.5 | 25.4 | 68.6 | 43.1 | 66.6 | 29.1 | 49.3 | 20.5 | 44.9 | 10.7 | 67.7 | 47.3 | 63.8 | 27.0 | 75.6 |

## C.4 Model Response

We present example responses from Spark-3.5 [56], GPT-4 [62], Qwen1.5-72B-Chat [41], LLaMA3-Chinese-8B-Chat [60], and Xunzi-Qwen1.5-7B [47] models on a subset of tasks, as illustrated in Figure 15, 16, 17, 18 and 19.

## C.5 Traditional Metrics

Traditional metrics for all evaluated models on the translation and punctuation tasks are listed in Table 10.

## C.6 The Analysis of Problem Difficulty and Performance

When constructing the benchmark, we have additionally annotated "difficulty" information for tasks within the field of Ancient Poetry, including Content Q&A, Title and Author Q&A, Write the Next Sentence, and Write the Previous Sentence. The difficulty levels are categorized into "Easy" and "Difficult". We have used the "Chinese Primary and Secondary School Curriculum" as the basis for classification, designating poems and verses covered in the curriculum as "Easy" and those that are uncommon, rare, and outside of the curriculum as "Difficult".

The performance of 31 LLMs on easy and difficult questions is illustrated in Table 11. We also present the variance in performance across different levels of question difficulty using box plots (Figure 14). The highest performance of LLMs on easy questions can reach up to 93.0, whereas the average score on difficult questions is only 2.1. From this, we can draw a preliminary conclusion that the performance of LLMs on knowledge-type tasks is highly dependent on the scope of the pre-training data. Most LLMs are trained with data that includes some common and simple ancient poetry content, but there is a severe lack of comprehensive and extensive ancient poetry data. It is worth mentioning that the Spark-3.5 model [56] achieved a score of 17.2 on difficult questions, significantly outperforming other models. As a general model, it has considerable potential in the field of ancient poetry.

Table 11: Performance of LLMs on easy and difficult problems. Zoom in for better view.

| Model | Overall | | Content Q&A | | Title and Author Q&A | | Write the Next Sentence | | Write the Previous Sentence | |
|---|---|---|---|---|---|---|---|---|---|---|
| | Simple | Hard | Simple | Hard | Simple | Hard | Simple | Hard | Simple | Hard |
| Baichuan2-7B-Chat [42] | 52.0 | 0.0 | 75.6 | 0.0 | 40.0 | 0.0 | 62.5 | 0.0 | 29.2 | 0.0 |
| Baichuan2-13B-Chat [42] | 61.5 | 0.3 | 92.8 | 0.0 | 48.3 | 0.4 | 79.2 | 0.0 | 20.8 | 1.3 |
| Firefly-Baichuan2-13B [54] | 71.2 | 0.1 | 97.7 | 0.2 | 52.5 | 0.0 | 91.7 | 0.0 | 50.0 | 0.0 |
| ChatGLM2-6B [43] | 38.7 | 0.4 | 54.5 | 0.1 | 25.0 | 0.4 | 62.5 | 0.0 | 20.8 | 1.3 |
| ChatGLM3-6B [43] | 36.7 | 0.6 | 50.9 | 0.0 | 19.2 | 0.4 | 66.7 | 0.0 | 25.0 | 2.6 |
| InternLM2-Chat-7B [55] | 79.1 | 4.2 | 83.7 | 0.0 | 75.7 | 1.3 | 91.7 | **13.2** | 66.7 | 9.2 |
| Qwen1.5-0.5B-Chat [41] | 2.9 | 0.0 | 4.5 | 0.0 | 4.2 | 0.0 | 0.0 | 0.0 | 0.0 | 0.0 |
| Qwen1.5-4B-Chat [41] | 67.2 | 0.5 | 97.7 | 0.0 | 65.8 | 0.0 | 66.7 | 0.0 | 16.7 | 2.6 |
| Qwen1.5-7B-Chat [41] | 79.1 | 1.4 | **100.0** | 0.1 | 67.5 | 0.8 | 83.3 | 1.3 | 66.7 | 5.3 |
| Qwen1.5-14B-Chat [41] | 80.3 | 5.3 | 99.4 | 0.0 | 82.5 | 0.0 | 79.2 | **13.2** | 41.7 | **18.4** |
| Qwen1.5-32B-Chat [41] | 87.4 | 3.0 | 98.8 | 0.8 | 75.8 | 0.4 | **100.0** | 3.9 | 83.3 | 11.2 |
| Qwen1.5-72B-Chat [41] | 85.4 | 4.5 | **100.0** | 0.0 | 73.3 | 0.0 | 95.8 | 10.5 | 79.2 | 15.8 |
| Yi-1.5-6B-Chat [52] | 67.3 | 1.9 | 78.1 | 0.0 | 58.3 | 0.3 | 87.5 | 3.9 | 50.0 | 6.6 |
| Yi-1.5-9B-Chat [52] | 74.9 | 3.8 | 82.7 | 0.0 | 69.2 | 2.9 | 87.5 | 9.2 | 62.5 | 7.9 |
| Yi-1.5-34B-Chat [52] | 87.9 | 2.7 | 92.8 | 0.1 | 86.4 | 2.1 | 95.8 | 7.9 | 75.0 | 3.9 |
| ERNIE-3.5-8K-0329 [10] | **93.0** | 5.8 | 99.7 | 0.6 | **86.7** | 2.8 | **100.0** | 9.2 | **89.6** | **18.4** |
| ERNIE-4.0-8K-0329 [10] | 91.1 | 5.1 | **100.0** | 0.4 | 82.5 | 1.0 | **100.0** | 10.5 | 87.5 | 17.1 |
| Spark-3.5 [56] | 83.7 | **17.2** | 99.7 | **46.1** | 77.5 | 1.2 | 91.7 | 3.9 | 62.5 | 0.0 |
| Gemma-1.1-7B-IT [57] | 2.6 | 0.0 | 0.0 | 0.0 | 6.7 | 0.0 | 0.0 | 0.0 | 0.0 | 0.0 |
| Ziya-LLaMA-13B-v1.1 [58] | 11.8 | 0.1 | 19.5 | 0.2 | 9.2 | 0.0 | 16.7 | 0.0 | 0.0 | 0.0 |
| LLaMA2-7B-Chat [40] | 0.7 | 0.3 | 0.0 | 0.0 | 1.7 | 1.0 | 0.0 | 0.0 | 0.0 | 0.0 |
| LLaMA2-13B-Chat [40] | 0.3 | 0.3 | 0.0 | 0.1 | 0.8 | 0.2 | 0.0 | 1.3 | 0.0 | 0.0 |
| LLaMA2-Chinese-7B-Chat [45] | 0.0 | 0.1 | 0.0 | 0.0 | 0.0 | 0.4 | 0.0 | 0.0 | 0.0 | 0.0 |
| LLaMA2-Chinese-13B-Chat [46] | 0.3 | 0.0 | 0.0 | 0.0 | 0.8 | 0.0 | 0.0 | 0.0 | 0.0 | 0.0 |
| LLaMA3-8B-Instruct [59] | 1.7 | 0.4 | 0.3 | 0.0 | 2.5 | 0.0 | 4.2 | 0.0 | 0.0 | 1.3 |
| LLaMA3-Chinese-8B-Chat [60] | 2.4 | 1.7 | 2.7 | 0.0 | 0.8 | **3.9** | 8.3 | 1.3 | 0.0 | 1.3 |
| GPT-3.5 [61] | 20.1 | 0.0 | 41.5 | 0.0 | 3.8 | 0.0 | 45.8 | 0.0 | 4.2 | 0.0 |
| GPT-4 [62] | 55.7 | 3.0 | 91.1 | 0.1 | 36.7 | 3.0 | 75.0 | 6.6 | 20.8 | 5.3 |
| Ancient-Chat-LLM-7B [51] | 54.5 | 0.7 | 88.6 | 0.0 | 26.9 | 2.4 | 75.0 | 0.0 | 41.7 | 0.0 |
| Bloom-7B-Chunhua [48] | 30.9 | 0.0 | 71.4 | 0.0 | 5.0 | 0.0 | 45.8 | 0.0 | 8.3 | 0.0 |
| Xunzi-Qwen1.5-7B [47] | 72.6 | 0.6 | 92.2 | 1.3 | 71.7 | 0.4 | 79.2 | 0.0 | 33.3 | 0.0 |
| Average | 48.2 | 2.1 | 61.8 | 1.6 | 40.5 | 0.8 | 57.8 | 3.1 | 33.4 | 4.2 |

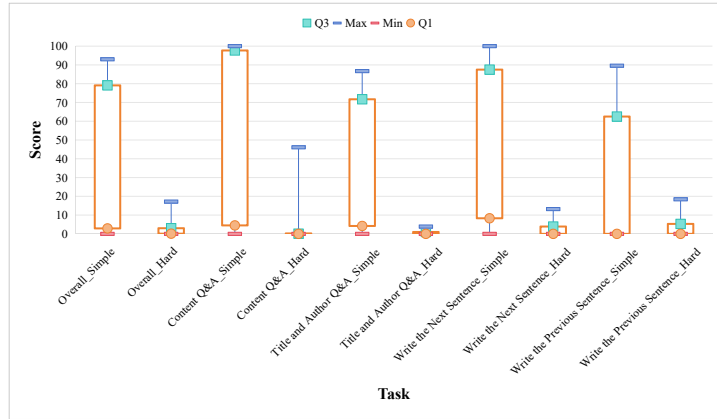

Figure 14: Box plot of LLMs' scores on easy and difficult questions. "Min" represents the minimum value; "Max" represents the maximum value; "Q3" represents the upper quartile; "Q1" represents the lower quartile.

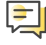 **Question:**
《小石潭记》中"斗折蛇行"是一句省略句，该句子省略了什么？
In the article "Record of the Little Stone Pool", "Twisted and turned like a snake" is an elliptical sentence. What is omitted in this sentence?

**Answer:**
该句子省略了"溪水"，完整表达为"（溪水）斗折蛇行"。
The sentence omits "stream". The complete expression is "(The stream) twisted and turned like a snake."

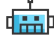 **Spark-3.5:**
《小石潭记》中的"斗折蛇行"省略了主语和谓语动词。完整的句子应该是"（其岸）斗折（而行），（如）蛇行"，意思是说岸边弯曲得像斗一样，像蛇那样曲折前行。
The phrase "Twisted and turned like a snake" in "Record of the Little Stone Pool" omits the subject and predicate verb. The complete sentence should be "(The bank) twisted (as it goes along), (like) a snake moving forward", meaning that the bank curves like a dipper and twists and turns forward like a snake. ❌

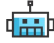 **GPT-4:**
"斗折蛇行"实际上是省略了主语和谓语，"斗折蛇行"的完整构成应该包含这条溪水是如何斗折蛇行的。若要补全句子，则可能变为"溪水斗折蛇行"或"溪流斗折蛇行"等表达。
"Twisted and turned like a snake" actually omits the subject and predicate. The complete construction of 'Twisted and turned like a snake' should include a description of how the stream twists and turns like a snake. To complete the sentence, it might become expressions such as 'the stream twisted and turned like a snake' or 'the creek twisted and turned like a snake.' ✅

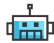 **Qwen1.5-72B-Chat:**
这句的完整表达应该是"（溪水）斗折蛇行"，其中省略了主语"溪水"。通过省略主语，作者使得句子更加简洁，但意思仍然清晰。
The complete expression of this sentence should be "(The stream) twisted and turned like a snake," in which the subject "stream" is omitted. By omitting the subject, the author makes the sentence more concise but the meaning remains clear. ✅

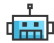 **LLaMA3-Chinese-8B-Chat:**
《小石潭记》是唐代诗人李白的作品，原文中并没有"斗折蛇行"这一句。可能是您所引用的文本有误，或者是后人对原文的误解或误记。如果您能提供更多的上下文或者是完整的句子，我可以帮助您分析这句话的含义和可能的省略内容。
"Record of the Little Stone Pool" is a work by the Tang Dynasty poet Li Bai, and there is no sentence "Twisted and turned like a snake" in the original text. It is possible that you have cited a misquoted text or that later generations misunderstood or misrecorded the original text. If you can provide more context or the complete sentence, I can help you analyze the meaning and possible omissions of this sentence. ❌

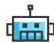 **Xunzi-Qwen1.5-7B:**
"斗折蛇行"是一句省略句，该句子省略了"溪水"。
"Twisted and turned like a snake" is an elliptical sentence, omitting the subject "stream". ✅

Figure 15: The sample responses of LLMs on the sentence structure task.

**Question:**
柳永写的《蝶恋花·伫倚危楼风细细》中，古诗词"为伊消得人憔悴"的上一句是什么？
*In Liu Yong's poem "Butterfly Loves Flower - Standing by the Tall Building with a Soft Breeze", what is the previous line of the ancient poem "I am thin and wan for her"?*

**Answer:**
衣带渐宽终不悔
*My belt grows loose, yet I never regret.*

**Spark-3.5:**
上一句是：衣带渐宽终不悔。
*The previous line is: "My belt grows loose, yet I never regret."* ✓

**GPT-4:**
柳永的《蝶恋花·伫倚危楼风细细》中，"为伊消得人憔悴"的上一句是"桃花尽日随流水"，完整的两句是：桃花尽日随流水，悠然见南山。为伊消得人憔悴，不得闲暇看山。
*In Liu Yong's poem "Butterfly Loves Flower - Standing by the Tall Building with a Soft Breeze", the line before "I am thin and wan for her" is "Peach blossoms float down the river all day long". The complete two lines are: "Peach blossoms float down the river all day long, leisurely gazing upon the South Mountain. I am thin and wan for her, too busy to gaze upon the mountains."* ✗

**QwenI.5-72B-Chat:**
在柳永的《蝶恋花·伫倚危楼风细细》中，"为伊消得人憔悴"的上一句是"衣带渐宽终不悔"。全句为：衣带渐宽终不悔，为伊消得人憔悴。
*In Liu Yong's poem "Butterfly Loves Flower - Standing by the Tall Building with a Soft Breeze", the line before "I am thin and wan for her" is "My belt grows loose, yet I never regret." The complete line is: "My belt grows loose, yet I never regret, I am thin and wan for her."* ✓

**LLaMA3-Chinese-8B-Chat:**
《蝶恋花·伫倚危楼风细细》这首诗是柳永的作品，原文如下：伫倚危楼风细细，望断人归无处归。为伊消得人憔悴，一夜风起梦难归。"为伊消得人憔悴"这句是整个诗句的第三句，前两句是"伫倚危楼风细细"和"望断人归无处归"。
*The poem "Butterfly Loves Flower - Standing by the Tall Building with a Soft Breeze" is a work of Liu Yong, and the original text is as follows: Standing by the tall building with a soft breeze, I look far into the distance but see no one returning. I am thin and wan for her, unable to return to my dreams with the wind rising overnight. "I am thin and wan for her" is the third line of the poem, and the first two lines are "Standing by the tall building with a soft breeze" and "I look far into the distance but see no one returning."* ✗

**Xunzi-QwenI.5-7B:**
伫倚危楼风细细。
*Standing by the tall building with a soft breeze.* ✗

Figure 16: The sample responses of LLMs on the basic Q&A task.

**Question:**
请问排在十二生肖中的第五位是什么动物？
*What is the fifth animal in the Chinese Zodiac?*

**Answer:**
龙
*Dragon*

**Spark-3.5:**
排在十二生肖中的第五位的动物是龙。龙作为十二生肖中第五位的动物，不仅是中国文化中的一个重要符号，也承载着丰富的民俗文化意义和传统价值观念。
*The animal that ranks fifth in the Chinese Zodiac is the dragon. As the fifth animal in the Chinese Zodiac, the dragon is not only an important symbol in Chinese culture, but also carries rich folklore meanings and traditional values.* ✓

**GPT-4:**
排在十二生肖中的第五位是龙。
*The fifth animal in the Chinese Zodiac is the dragon.* ✓

**QwenI.5-72B-Chat:**
排在十二生肖中的第五位是羊。
*The fifth animal in the Chinese Zodiac is the sheep.* ✗

**LLaMA3-Chinese-8B-Chat:**
十二生肖中的第五位是龙。
*The fifth position in the Chinese Zodiac is the dragon.* ✓

**Xunzi-QwenI.5-7B:**
排在十二生肖中的第五位是虎。
*The fifth animal in the Chinese Zodiac is the tiger.* ✗

Figure 17: The sample responses of LLMs on the knowledge of Sinology Q&A task.

**Question:**
补全下列歇后语：钉头碰钉子 ——
*Complete the following xiehouyu: The nail head hits the nail ——*

**Answer:**
硬碰硬
*Head-to-head confrontation*

**Spark-3.5:**
钉头碰钉子——硬碰硬，即两个都很强硬的人相互对着干。这个歇后语比喻双方都很厉害，互不相让，都要争个高低。在日常生活或工作中，我们可以用这个歇后语来形容两个人或团体之间的对抗，尤其是那些坚持己见、不肯妥协的情况。
*The nail head hits the nail —— head-to-head confrontation, meaning two very tough individuals are opposing each other. This xiehouyu metaphorically describes a situation where both parties are strong and unyielding, each determined to come out on top. In daily life or work, we can use this xiehouyu to describe a confrontation between two people or groups, especially in situations where both sides are stubborn and unwilling to compromise.* ✅

**GPT-4:**
钉头碰钉子——自有对头（意思是强的遇到强的，指遭遇实力相当的对手，碰到难对付的情况。）
*The nail head hits the nail —— an equal match (meaning a strong opponent meets another strong opponent, referring to encountering a rival of equal strength or a difficult situation).* ❌

**Qwen1.5-72B-Chat:**
钉头碰钉子——硬碰硬。
*The nail head hits the nail —— head-to-head confrontation.* ✅

**LLaMA3-Chinese-8B-Chat:**
钉头碰钉子——碰钉子。
*The nail head hits the nail —— hitting a nail.* ❌

**Xunzi-Qwen1.5-7B:**
钉头碰钉子——两败俱伤。
*The nail head hits the nail —— mutual destruction.* ❌

Figure 18: The sample responses of LLMs on the xiehouyu task.

**Question:**
把文言文翻译为白话文：守文直法，臣吏之节也。
*Translating classical Chinese into modern Chinese: "守文直法，臣吏之节也。"*

**Answer:**
遵守条文法律，是臣子和官吏的职责。
*It is the duty of courtiers and officials to comply with the provisions of the law.*

**Spark-3.5:**
遵守成文法令，是臣吏的节度。
*Compliance with written laws and decrees is the rule for officials.*

**GPT-4:**
翻译为白话文：坚守法度和遵循正直，是臣子和官吏应该遵守的原则。
*Translated into modern Chinese: Upholding laws and adhering to integrity are principles that should be followed by both courtiers and officials.*

**Qwen1.5-72B-Chat:**
遵守法律，秉直行事，这是臣子和官吏应有的节操。
*Obeying the law and acting with integrity are the moral integrity that courtiers and officials should have.*

**LLaMA3-Chinese-8B-Chat:**
遵守文明正法，这是臣子和官吏的品行准则。
*Abiding by the civilized and righteous law is the code of conduct for courtiers and officials.*

**Xunzi-Qwen1.5-7B:**
守法不屈，是臣子的节操。
*Remaining law-abiding and resolute is the moral integrity that courtiers should have.*

Figure 19: The sample responses of LLMs on the classical Chinese to modern Chinese task.

## C.7 The Analysis of Fine-Grained Tasks and Performance

Due to space limitations, the main body of the paper only presents the metrics of LLMs on 26 coarse-grained tasks. In fact, we have subdivided six of these tasks (T1-Sentence structure, T13-Appreciation, T14-Ancient poetry writing, T15-Basic Q&A, T21-Couplet, T22-Idiom) into subtasks, resulting in a total of 42 fine-grained tasks. Table 12 provides the performance of LLMs on these subtasks (fine-grained). Different subtasks involve more detailed knowledge points and methods of examination, hence there are significant performance differences among the subtasks. Overall, LLMs generally perform poorly on tasks involving elliptical sentence, content Q&A of ancient poetry, title and author Q&A of ancient poetry, write the previous sentence, and the origin of idiom. Researchers can use it as a reference to focus on improving the capabilities of LLMs in certain areas.

The task numbers presented in Table 12 correspond to the following meanings, with detailed information available in Appendix B.1.

T1-1: Inverted Sentence Structure; T1-2: Elliptical Sentence; T1-3: Inverted Sentence Types; T1-4: Sentence Structure Identification; T13-1: Appreciation Exam Questions; T13-2: Free Appreciation; T14-1: Poetry Writing; T14-2: Ci Writing; T14-3: Qu Writing; T15-1: Content Q&A; T15-2: Title and Author Q&A; T15-3: Write the Next Sentence; T15-4: Write the Previous Sentence; T15-5: Comprehension Dictation; T15-6: Genre Judgment; T21-1: Couplet Following; T21-2: Couplet Writing; T21-3: HengPi Writing; T22-1: Synonyms; T22-2: The Origin of Idiom; T22-3: Idiom Finding; T22-4: Idiom Explanation.

Table 12: Performance of LLMs on fine-grained tasks. Zoom in for better view.

| Model | Ancient Prose | | | | Ancient Poetry | | | | | | | | | | | Ancient Literary Culture | | | | | | |
|---|---|---|---|---|---|---|---|---|---|---|---|---|---|---|---|---|---|---|---|---|---|---|
| | T1-1 | T1-2 | T1-3 | T1-4 | T13-1 | T13-2 | T14-1 | T14-2 | T14-3 | T15-1 | T15-2 | T15-3 | T15-4 | T15-5 | T15-6 | T21-1 | T21-2 | T21-3 | T22-1 | T22-2 | T22-3 | T22-4 |
| Baichuan2-7B-Chat [42] | 20.4 | 10.9 | 0.0 | 60.5 | 50.4 | 60.7 | 52.3 | 59.0 | 69.0 | 16.2 | 12.0 | 15.0 | 7.0 | 49.4 | 31.7 | 52.0 | 59.6 | 69.8 | 36.7 | 2.5 | 24.6 | 52.3 |
| Baichuan2-13B-Chat [42] | 36.1 | 15.6 | 42.9 | 79.1 | 55.2 | 60.1 | 59.8 | 65.3 | 64.2 | 19.9 | 14.8 | 19.0 | 6.0 | 46.7 | 35.0 | 51.0 | 61.5 | 55.4 | 19.8 | 2.5 | 36.7 | 60.3 |
| Firefly-Baichuan2-13B [54] | 17.1 | 10.9 | 42.9 | 11.6 | 39.5 | 49.2 | 34.3 | 44.0 | 40.5 | 21.1 | 15.8 | 22.0 | 12.0 | 63.3 | 18.3 | 29.2 | 68.7 | 69.2 | 31.6 | 5.8 | 27.2 | 40.3 |
| ChatGLM2-6B [43] | 8.3 | 7.8 | 28.6 | 69.8 | 33.5 | 54.8 | 50.3 | 56.0 | 60.5 | 11.8 | 7.8 | 15.0 | 6.0 | 50.0 | 14.2 | 48.5 | 48.7 | 57.6 | 29.0 | 1.0 | 23.5 | 39.9 |
| ChatGLM3-6B [43] | 25.0 | 7.8 | 57.1 | 76.7 | 52.3 | 59.8 | 49.7 | 61.2 | 55.0 | 10.9 | 6.0 | 16.0 | 8.0 | 38.3 | 8.3 | 42.8 | 65.5 | 60.4 | 21.8 | 1.0 | 24.7 | 47.5 |
| InternLM2-Chat-7B [55] | 38.9 | 20.3 | 14.3 | 67.4 | 64.1 | 74.2 | 43.7 | 61.6 | 53.5 | 18.0 | 23.6 | 32.0 | 23.0 | 57.2 | 50.8 | 60.6 | 75.6 | 70.0 | 50.5 | 7.0 | 61.4 | 72.3 |
| Qwen1.5-0.5B-Chat [41] | 24.1 | 3.1 | 28.6 | 81.4 | 30.5 | 23.0 | 20.0 | 15.4 | 13.0 | 1.0 | 1.3 | 10.0 | 0.0 | 11.7 | 12.5 | 47.3 | 61.2 | 62.7 | 2.3 | 5.0 | 12.2 | 17.3 |
| Qwen1.5-4B-Chat [41] | 35.2 | 17.2 | 42.9 | 51.2 | 46.1 | 46.8 | 29.7 | 45.8 | 33.0 | 21.0 | 19.8 | 16.0 | 6.0 | 66.7 | 31.7 | 47.8 | 70.4 | 61.6 | 10.9 | 8.0 | 32.2 | 45.1 |
| Qwen1.5-7B-Chat [41] | 38.0 | 17.2 | 28.6 | 74.4 | 67.4 | 70.2 | 53.3 | 69.4 | 74.5 | 21.6 | 20.8 | 21.0 | 20.0 | 68.3 | 28.3 | 49.0 | 74.3 | 67.8 | 41.8 | 1.5 | 31.5 | 72.8 |
| Qwen1.5-14B-Chat [41] | 47.2 | 23.4 | 0.0 | 46.5 | 72.8 | 73.0 | 55.7 | 69.5 | 70.7 | 21.4 | 24.8 | 29.0 | 24.0 | 83.3 | 50.8 | 49.9 | 76.8 | 68.2 | 45.0 | 2.5 | 44.8 | 70.9 |
| Qwen1.5-32B-Chat [41] | 45.4 | 42.2 | 28.6 | 27.9 | 77.2 | 73.5 | 55.0 | 66.9 | 73.5 | 21.9 | 23.0 | 27.0 | 28.5 | 90.0 | 56.7 | 50.8 | 78.2 | 67.4 | 65.1 | 5.3 | 53.9 | 67.8 |
| Qwen1.5-72B-Chat [41] | 69.4 | 34.4 | 85.7 | 39.5 | 79.7 | 74.1 | 57.3 | 69.6 | 76.5 | 21.5 | 22.0 | 31.0 | 31.0 | 89.2 | 58.8 | 52.7 | 79.4 | 68.5 | 53.1 | 10.0 | 45.3 | 78.2 |
| Yi-1.5-6B-Chat [52] | 25.0 | 14.1 | 0.0 | 65.1 | 62.6 | 68.0 | 53.0 | 64.5 | 65.5 | 16.8 | 17.7 | 24.0 | 17.0 | 70.0 | 15.0 | 46.1 | 72.2 | 66.1 | 49.8 | 3.5 | 45.8 | 63.1 |
| Yi-1.5-9B-Chat [52] | 46.8 | 20.3 | 42.9 | 44.2 | 62.4 | 72.6 | 58.0 | 63.6 | 68.7 | 17.8 | 22.8 | 28.0 | 21.0 | 69.4 | 32.5 | 49.8 | 77.0 | 69.9 | 69.2 | 7.0 | 51.3 | 67.3 |
| Yi-1.5-34B-Chat [52] | 69.4 | 23.4 | 71.4 | 69.8 | 70.2 | 73.5 | 61.0 | 68.5 | 70.2 | 20.0 | 27.4 | 29.0 | 21.0 | 73.3 | 51.7 | 69.2 | 72.6 | 70.7 | 82.6 | 7.2 | 61.0 | 73.3 |
| ERNIE-3.5-8K-0329 [10] | 72.2 | 37.5 | 85.7 | 58.1 | 76.0 | 75.4 | 59.0 | 69.2 | 63.5 | 21.9 | 28.0 | 31.0 | 35.5 | 93.3 | 67.5 | 72.0 | 84.3 | 83.8 | 86.9 | 24.0 | 57.2 | 77.9 |
| ERNIE-4.0-8K-0329 [10] | 83.3 | 43.8 | 85.7 | 34.9 | 79.7 | 78.9 | 62.2 | 67.2 | 67.0 | 21.8 | 25.4 | 32.0 | 34.0 | 100.0 | 68.3 | 78.8 | 81.7 | 83.6 | 87.3 | 22.5 | 58.8 | 79.7 |
| Spark-3.5 [56] | 58.3 | 20.3 | 57.1 | 27.9 | 68.8 | 64.5 | 56.3 | 59.7 | 72.2 | 57.6 | 24.1 | 25.0 | 15.0 | 86.7 | 44.2 | 59.0 | 69.5 | 73.5 | 88.6 | 75.2 | 59.2 | 98.0 |
| Gemma-1.1-7B-IT [57] | 27.8 | 4.7 | 0.0 | 9.3 | 24.1 | 38.2 | 32.3 | 50.4 | 53.0 | 0.0 | 2.0 | 0.0 | 0.0 | 5.0 | 2.9 | 37.1 | 49.8 | 61.4 | 5.0 | 5.0 | 16.3 | 8.3 |
| Ziya-LLaMA-13B-v1.1 [58] | 30.6 | 0.0 | 71.4 | 60.5 | 43.9 | 49.1 | 39.0 | 53.8 | 57.0 | 4.3 | 2.8 | 4.0 | 0.0 | 26.7 | 18.3 | 44.5 | 53.2 | 52.8 | 13.2 | 5.0 | 29.8 | 48.8 |
| LLaMA2-7B-Chat [40] | 0.0 | 0.0 | 0.0 | 0.0 | 24.1 | 27.7 | 16.3 | 45.6 | 46.5 | 0.0 | 1.2 | 0.0 | 0.0 | 0.0 | 0.0 | 4.0 | 42.3 | 11.5 | 4.0 | 0.0 | 15.0 | 16.0 |
| LLaMA2-13B-Chat [40] | 10.2 | 3.1 | 0.0 | 9.3 | 27.6 | 33.2 | 26.0 | 49.8 | 67.0 | 1.0 | 4.0 | 10.0 | 0.0 | 5.6 | 1.7 | 26.5 | 53.3 | 21.8 | 5.2 | 5.0 | 18.0 | 27.8 |
| LLaMA2-Chinese-7B-Chat [45] | 2.8 | 0.0 | 0.0 | 7.0 | 16.1 | 27.2 | 16.7 | 25.6 | 5.0 | 0.0 | 3.0 | 0.0 | 0.0 | 0.0 | 0.0 | 10.0 | 30.9 | 31.0 | 4.5 | 0.0 | 8.3 | 10.9 |
| LLaMA2-Chinese-13B-Chat [46] | 18.5 | 1.6 | 14.3 | 0.0 | 24.3 | 32.1 | 19.7 | 24.4 | 20.2 | 0.0 | 3.0 | 0.0 | 0.0 | 6.7 | 2.1 | 30.7 | 30.1 | 57.7 | 3.6 | 0.0 | 5.5 | 17.2 |
| LLaMA3-8B-Instruct [59] | 15.3 | 1.6 | 0.0 | 69.8 | 44.0 | 51.6 | 31.0 | 57.2 | 68.2 | 1.0 | 8.0 | 20.0 | 10.0 | 3.3 | 6.7 | 46.1 | 64.2 | 67.7 | 24.2 | 5.0 | 22.8 | 41.5 |
| LLaMA3-Chinese-8B-Chat [60] | 22.2 | 0.0 | 0.0 | 69.8 | 47.7 | 55.8 | 46.3 | 54.4 | 51.0 | 6.0 | 3.0 | 3.0 | 10.0 | 1.7 | 8.0 | 52.7 | 49.7 | 66.6 | 24.7 | 5.0 | 25.0 | 37.1 |
| GPT-3.5 [61] | 22.2 | 15.6 | 14.3 | 72.1 | 43.4 | 52.0 | 44.3 | 57.8 | 64.0 | 8.9 | 3.0 | 11.0 | 10.0 | 6.7 | 10.8 | 38.2 | 51.7 | 52.6 | 5.2 | 1.5 | 14.8 | 44.8 |
| GPT-4 [62] | 52.8 | 18.8 | 57.1 | 16.3 | 60.8 | 69.4 | 62.7 | 64.7 | 68.7 | 19.7 | 13.1 | 23.0 | 9.0 | 13.3 | 19.7 | 45.4 | 73.9 | 56.4 | 49.5 | 4.0 | 36.3 | 64.8 |
| Ancient-Chat-LLM-7B [51] | 25.0 | 6.3 | 14.3 | 76.7 | 24.9 | 11.0 | 30.3 | 35.4 | 21.5 | 19.0 | 9.8 | 18.0 | 10.0 | 43.3 | 10.0 | 45.3 | 42.3 | 53.1 | 4.1 | 5.0 | 18.3 | 43.1 |
| Bloom-7B-Chunhua [48] | 13.9 | 3.1 | 57.1 | 23.3 | 23.4 | 35.6 | 45.7 | 37.8 | 33.0 | 15.4 | 1.5 | 11.0 | 2.0 | 23.9 | 13.3 | 41.3 | 48.8 | 52.4 | 17.1 | 3.5 | 25.7 | 42.8 |
| Xunzi-Qwen1.5-7B [47] | 26.9 | 15.6 | 0.0 | 65.1 | 39.0 | 18.2 | 46.0 | 39.6 | 36.0 | 20.8 | 21.8 | 19.0 | 8.0 | 76.7 | 1.7 | 52.7 | 44.7 | 55.9 | 23.9 | 1.0 | 40.0 | 30.8 |
| Average | 33.2 | 14.2 | 31.3 | 47.3 | 49.4 | 53.3 | 44.1 | 54.0 | 54.3 | 14.6 | 12.7 | 16.3 | 11.2 | 45.8 | 24.7 | 46.2 | 61.7 | 60.2 | 34.1 | 6.4 | 33.1 | 50.3 |

## C.8 Performance Radar Chart

Performance radar charts of all evaluated LLMs are depicted from Figure 20 through 50. The radar charts from left to right represent the fields of Ancient Prose, Ancient Poetry, and Ancient Literary Culture. The meanings of T1-T26 are detailed in Appendix B.1.

## C.9 Analysis of Model Scoring Preferences

We design two methods to analyze the preference bias in model scoring.

**Use another model to score.** GPT-3.5-Turbo-0125 [61] is used as an alternative scoring model for the five evaluated models. The results are shown in Table 13. The results show that the ranking of models remain the same as when scored by ERNIE-3.5-8K-0329 [10], with ERNIE-4.0-8K-0329 [10] still ranking first. This suggests that its leading position is genuine and less affected by scoring bias.

**Manual analysis.** We categorize the questions in WenMind into objective and subjective types, and conduct manual analysis on the model's scoring results for each. We find that for objective questions with standard answers, ERNIE essentially judges and scores each one strictly according to the scoring points of the answer, showing a weaker bias. For subjective questions, we select classical Chinese

writing and Qu writing tasks for analysis. (1) In classical Chinese writing task, ERNIE-4.0-8K-0329 scores 70.8, ranking second among all models. The first place is the Spark-3.5 [56] model with a score of 71.2. We review the response of ERNIE-4.0-8K-0329 and find that ERNIE-4.0-8K-0329 indeed has strong capabilities in classical Chinese writing, but this does not rule out the influence of model bias. (2) In the Qu writing task, ERNIE-4.0-8K-0329 scores 67, ranking 11th among all models. This also indirectly indicates that in subjective tasks, ERNIE does not necessarily give high scores to its own answers.

In summary, the preference bias in model scoring is relatively weak, making the use of model scoring a viable alternative to manual scoring and traditional metrics.

Table 13: The results of scoring five models using ERNIE-3.5-8K-0329 and GPT-3.5-Turbo-0125, respectively.

|  | ERNIE-3.5-8K-0329 | GPT-3.5-Turbo-0125 |
| --- | --- | --- |
| ERNIE-4.0-8K-0329[10] | 64.3 | 79.7 |
| Spark-3.5[56] | 60.9 | 77.4 |
| Qwen1.5-14B-Chat[41] | 54.9 | 73.7 |
| Baichuan2-13B-chat[42] | 45.5 | 66.4 |
| Xunzi-Qwen1.5-7B[47] | 37.0 | 59.8 |

### C.10 Error Analysis of LLMs

We observe significant performance differences in the model across different tasks. We conduct an error analysis on T1 (Sentence Structure), T13 (Appreciation), T14 (Ancient Poetry Writing), T15 (Basic Q&A), T21 (Couplet), and T22 (Idiom), which involve more specialized sub-tasks.

**T1-Sentence Structure.** The model performs weakest on Elliptical Sentence (T1-2), with a score of only 14.2. It struggles with identifying the omitted parts and restoring sentence completeness. In contrast, its performance on Sentence Structure Identification (T1-4) is better, with a score of 47.3, indicating a relative strength in recognizing simpler sentence structures. However, the model's overall understanding and application of various sentence types remain incomplete, especially when dealing with the implicit semantics and grammatical features of classical Chinese.

**T13-Appreciation.** The model achieves a higher score in Free Appreciation (T13-2) compared to Appreciation Exam Questions (T13-1), with scores of 53.3 and 49.4, respectively. This suggests that the model performs better in less restrictive contexts, showing adaptability in textual analysis. In structured exam questions, however, its limitations become apparent, likely due to the deeper cultural understanding and appreciation skills required, which the model has not yet fully mastered.

**T14-Ancient Poetry Writing.** The model scores well in Ci Writing (T14-2) and Qu Writing (T14-3), with scores of 54.0 and 54.3, indicating strong performance in writing tasks with fixed formats. However, its score for Poetry Writing (T14-1) is lower, at 44.1, reflecting the challenge of producing creative and artistically expressive poetry, particularly when dealing with diverse themes and styles.

**T15-Basic Q&A.** The lowest score is observed in Title and Author Q&A (T15-2), with only 12.7, highlighting challenges in recalling and recognizing specific textual information, particularly in large datasets. In contrast, Comprehension Dictation (T15-5) achieves the highest score, at 45.8, indicating the model's relative strength in language comprehension and summarization, despite its struggles with more specific questions.

**T21-Couplet.** The model excels in Couplet Writing (T21-2) and HengPi Writing (T21-3), scoring 61.7 and 60.2, respectively, demonstrating its capability in generating structured and formatted content. However, it performs less well in Couplet Following (T21-1), with a score of 46.2, indicating difficulties in completing couplets, where maintaining contextual harmony and creativity is crucial.

**T22-Idiom.** The lowest score is in The Origin of Idiom (T22-2), with only 6.4, revealing the model's significant shortcomings in associating idioms with their origins, reflecting a lack of historical knowledge. By contrast, it performs better in Idiom Explanation (T22-4), with a score of 50.3, showing that the model can provide accurate explanations for common idioms, though deeper cultural and contextual understanding still needs improvement.

In summary, the model performs well in tasks with more rigid rules and structures but struggles in tasks requiring a strong cultural background and high creativity, highlighting areas for future improvement.

## C.11 Traditional Chinese Version of WenMind

We additionally provide a traditional Chinese version of the WenMind dataset to evaluate the performance of LLMs in the context of traditional Chinese within the CCLLA field. Table 14, 15, and 16 show the evaluation results for the traditional Chinese version of WenMind. The Spark-3.5 model [56] had been discontinued at the time of the traditional Chinese evaluation experiments, so no corresponding evaluation results are available.

Below is a brief analysis of the evaluation results:

**Overall Performance and Trends.** The average score for the traditional Chinese version is 36.7, which is lower than the 41.2 for the simplified Chinese version. This gap may reflect limitations in the model's performance when processing traditional Chinese texts, potentially influenced by the language environment and modern usage frequency. The main tasks focus on classical texts and ancient poetry, indicating the advantage of traditional Chinese in traditional cultural domains. However, its performance is somewhat lacking in modern applications and data support.

**Model Performance.** In the traditional Chinese version, ERNIE-4.0-8K-0329 [10] leads with a total score of 58.4, demonstrating strong performance in handling tasks related to traditional Chinese. In the simplified Chinese version, this model also excels, with a total score of 64.3, maintaining a top position. This reflects ERNIE-4.0-8K-0329's consistently high performance across both versions. In contrast, LLaMA2-7B-Chat [40] scores only 14.2 in the traditional Chinese evaluation, making it the weakest model. In the simplified Chinese version, this model also scores poorly, with a score of 13.0. This indicates LLaMA2-7B-Chat's consistent underperformance in both language versions, likely due to insufficient targeted training data to handle the complexity and diversity of the language.

**Parameter Scale and Model Efficiency.** As the model parameter size increases, there is a noticeable improvement in performance. For example, the performance of the Qwen1.5 series [41] improves with an increase in parameters, particularly in complex text tasks. This suggests that large-scale models are better at handling traditional Chinese tasks by offering enhanced data processing and feature extraction capabilities, providing performance improvements across various tasks. This conclusion is consistent with the findings for the simplified Chinese version.

Table 14: Results of all evaluated models on the traditional Chinese version of WenMind. Zoom in for better view.

| Model | Overall | Ancient Prose | | | | | | | | | | | | Ancient Poetry | | | | | | | | Ancient Literary Culture | | | | | |
|---|---|---|---|---|---|---|---|---|---|---|---|---|---|---|---|---|---|---|---|---|---|---|---|---|---|---|---|
| | | T1 | T2 | T3 | T4 | T5 | T6 | T7 | T8 | T9 | T10 | T11 | T12 | T13 | T14 | T15 | T16 | T17 | T18 | T19 | T20 | T21 | T22 | T23 | T24 | T25 | T26 |
| Baichuan2-7B-Chat [42] | 30.0 | 17.7 | 48.1 | 53.3 | 53.6 | 38.5 | 23.8 | 23.2 | 49.6 | 35.0 | 31.5 | 35.5 | 34.7 | 20.3 | 42.5 | 2.8 | 49.4 | 54.0 | 53.0 | 63.8 | 10.4 | 43.3 | 18.5 | 8.0 | 8.0 | 24.0 | 41.5 |
| Baichuan2-13B-Chat [42] | 43.2 | 48.5 | 59.3 | 59.9 | 66.6 | 66.5 | 42.0 | 39.5 | 61.4 | 40.0 | 23.5 | 50.5 | 57.7 | 45.6 | 49.0 | 16.1 | 54.4 | 49.0 | 50.2 | 19.0 | 45.6 | 57.8 | 45.3 | 13.0 | 18.5 | 38.0 | 59.9 |
| Firefly-Baichuan2-13B [54] | 38.5 | 23.7 | 61.0 | 62.2 | 65.9 | 74.0 | 34.0 | 35.3 | 57.2 | 31.0 | 23.5 | 49.5 | 56.5 | 32.0 | 35.4 | 15.0 | 59.6 | 30.5 | 51.9 | 13.2 | 18.6 | 55.0 | 41.2 | 5.0 | 9.0 | 32.0 | 58.6 |
| ChatGLM2-6B [43] | 33.0 | 35.3 | 47.9 | 56.4 | 63.7 | 50.2 | 30.0 | 37.0 | 53.0 | 23.0 | 4.5 | 34.0 | 58.0 | 34.3 | 41.1 | 10.1 | 47.1 | 23.0 | 36.1 | 12.2 | 29.4 | 57.8 | 34.0 | 9.0 | 4.5 | 20.0 | 44.1 |
| ChatGLM3-6B [43] | 35.1 | 43.2 | 47.0 | 58.6 | 49.6 | 58.2 | 31.0 | 41.3 | 52.1 | 33.0 | 26.0 | 36.0 | 60.9 | 36.4 | 41.0 | 8.4 | 45.2 | 36.4 | 43.3 | 13.2 | 30.1 | 60.8 | 37.0 | 7.0 | 4.0 | 31.0 | 36.5 |
| InternLM2-Chat-7B [55] | 43.9 | 28.2 | 56.0 | 56.9 | 67.3 | 73.8 | 45.5 | 49.5 | 70.2 | 43.0 | 26.5 | 49.0 | 62.6 | 45.0 | 46.9 | 16.5 | 56.7 | 48.3 | 55.8 | 26.0 | 49.4 | 61.4 | 47.3 | 8.0 | 10.5 | 26.0 | 58.5 |
| Qwen1.5-0.5B-Chat [41] | 30.9 | 48.5 | 49.7 | 56.0 | 65.4 | 57.2 | 36.0 | 18.7 | 42.2 | 28.0 | 19.0 | 42.0 | 50.6 | 21.5 | 42.6 | 4.5 | 55.7 | 35.5 | 26.2 | 1.6 | 3.8 | 55.6 | 29.2 | 0.0 | 7.0 | 22.0 | 15.4 |
| Qwen1.5-4B-Chat [41] | 36.9 | 34.0 | 49.8 | 57.2 | 54.6 | 69.9 | 31.0 | 25.2 | 50.5 | 48.0 | 19.0 | 52.0 | 55.3 | 33.7 | 36.6 | 16.0 | 50.3 | 46.3 | 43.2 | 10.7 | 20.4 | 60.3 | 31.2 | 2.0 | 8.5 | 27.0 | 54.2 |
| Qwen1.5-7B-Chat [41] | 42.8 | 40.0 | 51.0 | 55.6 | 53.4 | 68.8 | 40.0 | 59.2 | 76.5 | 61.0 | 13.0 | 55.5 | 65.1 | 54.1 | 54.6 | 15.5 | 52.7 | 59.0 | 51.6 | 26.1 | 46.8 | 55.9 | 35.1 | 13.0 | 9.0 | 33.0 | 57.5 |
| Qwen1.5-14B-Chat [41] | 49.3 | 31.3 | 57.3 | 57.8 | 54.8 | 73.6 | 45.0 | 63.8 | 75.3 | 69.0 | 29.0 | 61.0 | 68.8 | 58.8 | 57.2 | 23.4 | 58.8 | 62.0 | 59.1 | 31.2 | 62.3 | 62.9 | 49.7 | 11.0 | 14.5 | 33.0 | 72.1 |
| Qwen1.5-32B-Chat [41] | 50.7 | 30.3 | 62.3 | 62.6 | 54.6 | 77.0 | 42.5 | 59.0 | 71.5 | 64.0 | 30.5 | 68.5 | 69.5 | 61.3 | 54.5 | 24.8 | 57.9 | 58.7 | 57.0 | 38.1 | 65.5 | 61.1 | 48.6 | 23.5 | 16.0 | 46.0 | 76.5 |
| Qwen1.5-72B-Chat [41] | 53.4 | 40.5 | 60.2 | 61.1 | 55.9 | 79.4 | 50.0 | 72.0 | 79.8 | 67.0 | 39.5 | 71.0 | 69.8 | 64.2 | 56.9 | 25.4 | 63.3 | 64.5 | 58.1 | 36.5 | 64.6 | 62.1 | 53.1 | 24.0 | 13.0 | 43.0 | 84.1 |
| Yi-1.5-6B-Chat [52] | 43.0 | 30.3 | 55.7 | 61.5 | 61.3 | 66.2 | 42.0 | 61.8 | 69.8 | 53.0 | 19.0 | 49.0 | 66.6 | 45.2 | 51.1 | 11.9 | 57.2 | 43.8 | 51.5 | 30.0 | 54.4 | 63.4 | 46.3 | 7.0 | 2.5 | 26.0 | 62.3 |
| Yi-1.5-9B-Chat [52] | 43.9 | 29.5 | 60.0 | 58.6 | 49.4 | 68.3 | 44.0 | 70.0 | 73.8 | 58.0 | 24.0 | 58.0 | 64.5 | 46.1 | 51.3 | 12.8 | 58.4 | 51.9 | 54.9 | 27.6 | 47.3 | 60.8 | 45.4 | 8.0 | 6.5 | 30.0 | 63.0 |
| Yi-1.5-34B-Chat [52] | 51.8 | 35.0 | 66.1 | 60.3 | 54.5 | 72.9 | 53.0 | 68.0 | 79.8 | 76.0 | 31.0 | 62.0 | 66.4 | 55.4 | 52.1 | 24.6 | 63.6 | 59.7 | 58.2 | 38.3 | 73.3 | 65.9 | 49.8 | 15.0 | 8.0 | 47.0 | 79.5 |
| ERNIE-3.5-8K-0329 [10] | 56.7 | 53.2 | 62.7 | 75.3 | 64.8 | 81.4 | 43.3 | 72.0 | 79.3 | 64.0 | 36.5 | 72.5 | 64.0 | 58.1 | 49.9 | 33.4 | 63.4 | 59.4 | 57.5 | 34.8 | 65.7 | 75.1 | 53.8 | 68.0 | 43.0 | 57.0 | 85.2 |
| ERNIE-4.0-8K-0329 [10] | 58.4 | 40.2 | 62.8 | 52.9 | 62.4 | 82.4 | 53.8 | 70.5 | 75.3 | 74.0 | 48.0 | 74.5 | 64.2 | 61.0 | 50.5 | 32.9 | 59.0 | 65.9 | 58.0 | 25.1 | 65.5 | 70.8 | 55.3 | 84.0 | 67.5 | 72.0 | 88.3 |
| Gemma-1.1-7B-IT [57] | 23.3 | 8.1 | 24.4 | 42.1 | 41.9 | 29.5 | 26.0 | 16.8 | 27.9 | 18.0 | 17.0 | 35.5 | 55.8 | 19.6 | 36.1 | 0.7 | 24.3 | 46.0 | 38.2 | 1.1 | 16.1 | 52.3 | 27.1 | 2.0 | 0.0 | 20.0 | 5.6 |
| Ziya-LLaMA-13B-v1.1 [58] | 31.4 | 35.6 | 45.2 | 61.7 | 16.8 | 56.0 | 27.5 | 33.8 | 53.9 | 31.0 | 26.5 | 31.0 | 54.8 | 23.8 | 33.5 | 4.5 | 49.0 | 42.5 | 40.4 | 11.2 | 28.1 | 58.8 | 37.9 | 3.0 | 4.5 | 16.0 | 34.0 |
| LLaMA2-7B-Chat [40] | 14.2 | 3.0 | 1.9 | 2.8 | 15.9 | 0.6 | 14.5 | 2.0 | 23.3 | 10.0 | 1.0 | 8.0 | 53.9 | 22.1 | 29.7 | 0.2 | 17.4 | 32.2 | 29.6 | 0.0 | 6.3 | 65.2 | 23.0 | 0.0 | 0.0 | 5.0 | 2.5 |
| LLaMA2-13B-Chat [40] | 20.2 | 5.0 | 14.2 | 23.1 | 30.2 | 19.1 | 12.6 | 14.8 | 40.9 | 20.0 | 3.0 | 15.0 | 57.3 | 22.1 | 32.9 | 1.4 | 36.0 | 38.5 | 40.6 | 3.0 | 22.1 | 53.7 | 29.5 | 2.0 | 0.0 | 10.0 | 5.4 |
| LLaMA2-Chinese-7B-Chat [45] | 19.5 | 4.7 | 13.7 | 38.5 | 51.8 | 19.8 | 17.2 | 13.5 | 28.2 | 20.0 | 2.5 | 23.5 | 56.1 | 15.2 | 36.0 | 0.2 | 21.3 | 18.4 | 30.5 | 2.8 | 11.3 | 53.3 | 28.8 | 0.0 | 0.0 | 19.0 | 3.1 |
| LLaMA2-Chinese-13B-Chat [46] | 21.1 | 9.7 | 27.7 | 39.6 | 42.8 | 14.3 | 14.5 | 15.7 | 34.9 | 21.0 | 12.5 | 25.0 | 52.0 | 19.1 | 31.6 | 0.7 | 27.0 | 27.3 | 38.1 | 3.1 | 12.4 | 51.1 | 30.3 | 0.0 | 0.0 | 13.0 | 11.5 |
| LLaMA3-8B-Instruct [59] | 26.8 | 26.8 | 41.8 | 50.2 | 41.5 | 48.6 | 43.5 | 42.7 | 56.7 | 39.0 | 13.5 | 44.0 | 57.4 | 30.6 | 30.0 | 1.8 | 42.9 | 45.0 | 46.4 | 4.0 | 35.9 | 57.3 | 33.5 | 4.0 | 0.0 | 24.0 | 22.1 |
| LLaMA3-Chinese-8B-Chat [60] | 35.8 | 36.3 | 43.0 | 60.4 | 47.4 | 70.2 | 46.0 | 49.5 | 61.1 | 41.0 | 22.5 | 49.5 | 61.9 | 35.8 | 39.0 | 1.1 | 51.9 | 43.3 | 47.3 | 5.7 | 43.8 | 55.4 | 39.7 | 3.0 | 2.0 | 21.0 | 21.8 |
| GPT-3.5 [61] | 37.7 | 37.5 | 47.4 | 60.5 | 47.2 | 71.1 | 43.0 | 38.3 | 55.3 | 34.0 | 25.5 | 49.0 | 62.2 | 43.9 | 43.5 | 4.5 | 50.9 | 50.5 | 55.9 | 9.1 | 37.5 | 58.4 | 41.8 | 6.0 | 0.0 | 31.0 | 36.3 |
| GPT-4 [62] | 46.2 | 17.0 | 56.2 | 62.3 | 51.4 | 73.1 | 47.5 | 65.8 | 80.2 | 52.0 | 33.0 | 69.5 | 67.9 | 55.6 | 51.1 | 11.1 | 58.0 | 57.0 | 56.6 | 22.8 | 60.4 | 61.4 | 46.0 | 17.0 | 7.0 | 39.0 | 63.6 |
| Ancient-Chat-LLM-7B [51] | 15.9 | 8.0 | 4.5 | 2.2 | 52.2 | 2.4 | 7.5 | 5.0 | 13.3 | 13.0 | 21.0 | 20.0 | 36.8 | 9.0 | 31.9 | 5.0 | 18.7 | 5.5 | 4.7 | 2.7 | 2.6 | 67.2 | 25.6 | 0.0 | 1.0 | 9.0 | 11.0 |
| Bloom-7B-Chunhua [48] | 25.8 | 28.5 | 25.4 | 23.2 | 51.1 | 75.1 | 20.5 | 27.2 | 43.3 | 31.0 | 8.0 | 39.5 | 27.4 | 21.9 | 17.6 | 6.5 | 43.8 | 39.5 | 41.2 | 11.5 | 9.8 | 41.0 | 16.1 | 4.0 | 0.0 | 19.0 | 37.9 |
| Xunzi-Qwen1.5-7B [47] | 35.3 | 40.2 | 59.4 | 51.7 | 26.2 | 81.5 | 30.5 | 22.5 | 53.3 | 53.0 | 18.5 | 49.0 | 63.1 | 36.0 | 49.5 | 16.8 | 57.7 | 8.0 | 44.2 | 14.9 | 0.7 | 55.9 | 32.9 | 6.0 | 7.5 | 37.0 | 52.7 |
| Average | 36.7 | 29.0 | 45.4 | 50.1 | 50.5 | 57.3 | 34.6 | 40.5 | 56.3 | 41.7 | 21.6 | 46.0 | 58.1 | 37.6 | 42.5 | 11.6 | 48.4 | 43.4 | 46.0 | 18.0 | 34.7 | 58.7 | 37.8 | 11.8 | 9.4 | 29.2 | 44.8 |

# D  Data Access

Our benchmark and related code are available at `https://github.com/SCUT-DLVCLab/WenMind`.

Table 15: Results of all evaluated models on different domains and capabilities in the traditional Chinese version of WenMind.

| Model | Overall | Domain | | | Capability | | |
|---|---|---|---|---|---|---|---|
| | | Ancient Prose | Ancient Poetry | Ancient Literary Culture | Understanding | Generation | Knowledge |
| Baichuan2-7B-Chat [42] | 30.0 | 38.4 | 23.7 | 26.4 | 39.1 | 41.5 | 14.8 |
| Baichuan2-13B-Chat [42] | 43.2 | 51.8 | 33.6 | 44.4 | 50.9 | 56.0 | 29.6 |
| Firefly-Baichuan2-13B [54] | 38.5 | 49.7 | 26.2 | 40.0 | 46.9 | 51.4 | 23.9 |
| ChatGLM2-6B [43] | 33.0 | 41.0 | 23.2 | 35.4 | 38.6 | 54.5 | 19.7 |
| ChatGLM3-6B [43] | 35.1 | 44.4 | 24.4 | 37.1 | 42.3 | 56.8 | 19.8 |
| InternLM2-Chat-7B [55] | 43.9 | 52.8 | 34.8 | 43.7 | 51.6 | 58.7 | 29.5 |
| Qwen1.5-0.5B-Chat [41] | 30.9 | 44.1 | 18.1 | 29.5 | 41.5 | 52.0 | 11.1 |
| Qwen1.5-4B-Chat [41] | 36.9 | 46.3 | 27.4 | 36.6 | 44.9 | 54.6 | 21.4 |
| Qwen1.5-7B-Chat [41] | 42.8 | 51.4 | 36.4 | 38.8 | 51.8 | 57.5 | 26.8 |
| Qwen1.5-14B-Chat [41] | 49.3 | 56.1 | 43.4 | 47.8 | 56.6 | 62.9 | 36.1 |
| Qwen1.5-32B-Chat [41] | 50.7 | 57.4 | 44.4 | 49.8 | 57.4 | 61.5 | 38.9 |
| Qwen1.5-72B-Chat [41] | 53.4 | 61.2 | 46.2 | 52.0 | 61.5 | 62.6 | 40.1 |
| Yi-1.5-6B-Chat [52] | 43.0 | 52.2 | 33.3 | 43.5 | 50.6 | 61.6 | 28.0 |
| Yi-1.5-9B-Chat [52] | 43.9 | 53.7 | 34.1 | 43.4 | 52.8 | 59.6 | 27.9 |
| Yi-1.5-34B-Chat [52] | 51.8 | 59.2 | 44.9 | 50.5 | 58.9 | 63.2 | 39.2 |
| ERNIE-3.5-8K-0329 [10] | 56.7 | 61.3 | 47.7 | 63.7 | 60.8 | 67.9 | 48.2 |
| ERNIE-4.0-8K-0329 [10] | 58.4 | 63.0 | 47.5 | 68.3 | 62.6 | 65.4 | 50.9 |
| Gemma-1.1-7B-IT [57] | 23.3 | 29.4 | 15.2 | 26.1 | 28.5 | 49.7 | 9.3 |
| Ziya-LLaMA-13B-v1.1 [58] | 31.4 | 38.9 | 21.4 | 35.0 | 37.9 | 52.9 | 16.9 |
| LLaMA2-7B-Chat [40] | 14.2 | 9.6 | 11.5 | 26.2 | 11.9 | 55.8 | 6.1 |
| LLaMA2-13B-Chat [40] | 20.2 | 19.6 | 16.9 | 26.4 | 21.6 | 50.3 | 10.2 |
| LLaMA2-Chinese-7B-Chat [45] | 19.5 | 24.0 | 10.5 | 26.4 | 21.3 | 50.4 | 8.7 |
| LLaMA2-Chinese-13B-Chat [46] | 21.1 | 25.6 | 12.9 | 26.8 | 24.4 | 47.4 | 9.6 |
| LLaMA3-8B-Instruct [59] | 31.6 | 41.5 | 21.1 | 32.1 | 40.3 | 51.9 | 14.7 |
| LLaMA3-Chinese-8B-Chat [60] | 35.8 | 48.8 | 23.7 | 33.5 | 46.8 | 53.4 | 16.4 |
| GPT-3.5 [61] | 37.7 | 48.2 | 26.9 | 37.8 | 47.7 | 56.2 | 19.4 |
| GPT-4 [62] | 46.2 | 56.3 | 36.2 | 45.6 | 55.9 | 60.6 | 29.4 |
| Ancient-Chat-LLM-7B [51] | 15.9 | 15.6 | 8.2 | 29.1 | 13.3 | 54.0 | 9.2 |
| Bloom-7B-Chunhua [48] | 25.8 | 33.8 | 18.4 | 24.4 | 34.3 | 33.6 | 12.4 |
| Xunzi-Qwen1.5-7B [47] | 35.3 | 45.6 | 23.7 | 37.0 | 41.8 | 56.0 | 21.0 |
| Average | 36.7 | 44.0 | 27.9 | 38.6 | 43.2 | 55.3 | 23.0 |

Table 16: Traditional metrics on translation and punctuation tasks in the traditional Chinese version of WenMind. See Appendix B.1 for details on the tasks represented by T2, T3, T5, T16, and T18. Zoom in for better view.

| Model | T2 ROUGE-1 | T2 ROUGE-2 | T2 ROUGE-L | T2 BLEU | T3 ROUGE-1 | T3 ROUGE-2 | T3 ROUGE-L | T3 BLEU | T16 ROUGE-1 | T16 ROUGE-2 | T16 ROUGE-L | T16 BLEU | T18 ROUGE-1 | T18 ROUGE-2 | T18 ROUGE-L | T18 BLEU | T5 F1 |
|---|---|---|---|---|---|---|---|---|---|---|---|---|---|---|---|---|---|
| Baichuan2-7B-Chat [42] | 38.1 | 15.3 | 36.4 | 6.8 | 41.5 | 17.3 | 39.6 | 5.9 | 29.7 | 9.9 | 26.5 | 3.8 | 44.5 | 23.5 | 40.1 | 8.3 | 18.4 |
| Baichuan2-13B-Chat [42] | 49.4 | 21.4 | 46.5 | 9.8 | 57.7 | 28.3 | 55.8 | 13.5 | 49.1 | 19.4 | 40.4 | 7.9 | 78.8 | 57.2 | 73.3 | 37.5 | 50.7 |
| Firefly-Baichuan2-13B [54] | 52.7 | 24.2 | 50.9 | 10.9 | 53.9 | 25.5 | 52.4 | 11.9 | 50.6 | 19.7 | 47.4 | 8.6 | 78.5 | 56.4 | 72.9 | 35.5 | 50.5 |
| ChatGLM2-6B [43] | 44.6 | 18.4 | 38.9 | 6.2 | 41.8 | 16.8 | 40.4 | 4.4 | 41.0 | 12.9 | 33.3 | 4.3 | 69.1 | 50.8 | 63.7 | 28.8 | 33.9 |
| ChatGLM3-6B [43] | 47.8 | 20.3 | 43.9 | 8.1 | 48.5 | 20.6 | 47.0 | 7.7 | 44.2 | 14.9 | 36.0 | 6.1 | 71.8 | 53.4 | 66.4 | 32.7 | 39.8 |
| InternLM2-Chat-7B [55] | 48.8 | 22.5 | 45.1 | 9.4 | 56.0 | 28.5 | 53.9 | 13.4 | 47.9 | 17.8 | 38.4 | 6.2 | 76.3 | 55.0 | 70.5 | 31.7 | 58.9 |
| Qwen1.5-0.5B-Chat [41] | 51.2 | 22.3 | 49.1 | 10.3 | 51.9 | 23.8 | 50.3 | 9.8 | 48.7 | 18.6 | 45.3 | 7.9 | 69.3 | 47.4 | 64.4 | 25.2 | 11.2 |
| Qwen1.5-4B-Chat [41] | 56.4 | 27.9 | 53.7 | 16.6 | 58.0 | 28.8 | 56.3 | 14.0 | 51.5 | 20.1 | 47.3 | 8.6 | 76.5 | 54.1 | 72.0 | 33.0 | 48.5 |
| Qwen1.5-7B-Chat [41] | 45.1 | 18.1 | 42.2 | 8.0 | 53.0 | 23.2 | 49.9 | 9.9 | 42.5 | 13.2 | 35.0 | 6.3 | 75.7 | 55.5 | 69.5 | 33.1 | 55.4 |
| Qwen1.5-14B-Chat [41] | 52.9 | 24.3 | 49.7 | 13.1 | 54.4 | 23.9 | 51.5 | 11.1 | 46.6 | 15.3 | 37.9 | 7.2 | 72.7 | 53.7 | 68.2 | 32.2 | 61.4 |
| Qwen1.5-32B-Chat [41] | 54.4 | 26.8 | 51.6 | 14.3 | 59.2 | 30.7 | 57.1 | 15.8 | 45.3 | 15.7 | 37.8 | 7.6 | 76.0 | 56.3 | 71.0 | 36.5 | 67.2 |
| Qwen1.5-72B-Chat [41] | 51.3 | 22.9 | 48.0 | 11.7 | 58.0 | 29.0 | 55.8 | 14.4 | 46.6 | 15.4 | 37.7 | 7.4 | 77.3 | 57.7 | 73.1 | 36.7 | 70.4 |
| Yi-1.5-6B-Chat [52] | 53.0 | 26.8 | 49.0 | 14.0 | 50.6 | 21.7 | 48.3 | 8.3 | 47.8 | 16.4 | 38.3 | 6.2 | 75.1 | 53.1 | 69.8 | 31.2 | 45.2 |
| Yi-1.5-9B-Chat [52] | 50.8 | 22.8 | 47.1 | 9.9 | 56.8 | 27.5 | 54.5 | 12.4 | 48.2 | 17.2 | 38.4 | 6.9 | 78.1 | 56.5 | 71.8 | 34.1 | 56.6 |
| Yi-1.5-34B-Chat [52] | 50.0 | 24.5 | 45.6 | 10.3 | 52.3 | 24.9 | 50.6 | 8.9 | 48.3 | 18.4 | 37.6 | 7.0 | 77.6 | 56.8 | 71.5 | 35.7 | 60.4 |
| ERNIE-3.5-8K-0329 [10] | 56.1 | 26.2 | 53.8 | 12.2 | 59.2 | 31.1 | 57.4 | 16.3 | 43.4 | 15.1 | 31.1 | 3.9 | 75.7 | 56.3 | 69.0 | 35.8 | 69.3 |
| ERNIE-4.0-8K-0329 [10] | 53.3 | 24.7 | 50.8 | 11.1 | 62.6 | 37.1 | 61.0 | 23.5 | 45.0 | 15.0 | 35.7 | 5.7 | 69.4 | 52.7 | 64.4 | 31.6 | 63.7 |
| Gemma-1.1-7b-IT [57] | 35.6 | 12.1 | 32.5 | 3.6 | 39.2 | 14.2 | 37.2 | 2.7 | 36.0 | 9.6 | 30.0 | 3.0 | 73.9 | 53.5 | 68.8 | 29.0 | 18.1 |
| Ziya-LLaMA-13B-v1.1 [58] | 40.8 | 17.3 | 39.1 | 7.4 | 27.5 | 8.8 | 26.6 | 2.5 | 41.0 | 13.3 | 35.9 | 5.6 | 77.5 | 55.5 | 72.6 | 33.6 | 46.5 |
| LLaMA2-7B-Chat [40] | 14.9 | 2.0 | 14.3 | 0.0 | 11.8 | 1.5 | 11.4 | 0.0 | 20.0 | 3.6 | 18.7 | 0.1 | 72.9 | 52.4 | 66.5 | 16.9 | 0.1 |
| LLaMA2-13B-Chat [40] | 18.8 | 2.9 | 18.1 | 0.0 | 22.0 | 3.6 | 21.4 | 0.0 | 32.1 | 6.4 | 30.0 | 0.2 | 72.4 | 52.5 | 65.8 | 17.5 | 0.7 |
| LLaMA2-Chinese-7B-Chat [45] | 25.9 | 8.2 | 23.1 | 0.9 | 30.3 | 10.6 | 29.1 | 1.9 | 41.6 | 14.2 | 37.1 | 4.6 | 72.0 | 52.5 | 66.8 | 25.3 | 10.0 |
| LLaMA2-Chinese-13B-Chat [46] | 28.6 | 9.1 | 25.5 | 1.6 | 25.7 | 8.9 | 24.7 | 1.6 | 11.4 | 2.8 | 10.2 | 0.5 | 74.2 | 53.4 | 67.5 | 29.9 | 9.5 |
| LLaMA3-8B-Instruct [59] | 31.5 | 8.0 | 28.5 | 0.4 | 30.5 | 8.1 | 29.2 | 0.6 | 39.7 | 8.3 | 36.0 | 0.6 | 57.2 | 48.4 | 53.8 | 12.1 | 9.8 |
| LLaMA3-Chinese-8B-Chat [60] | 50.1 | 21.8 | 46.3 | 11.1 | 55.7 | 25.5 | 53.7 | 11.0 | 46.1 | 14.5 | 53.7 | 6.1 | 76.1 | 56.8 | 71.4 | 34.3 | 53.4 |
| GPT-3.5 [61] | 50.2 | 22.2 | 46.6 | 11.2 | 53.8 | 24.0 | 51.9 | 9.6 | 46.9 | 16.2 | 38.4 | 7.6 | 78.7 | 57.4 | 72.6 | 37.1 | 58.9 |
| GPT-4 [62] | 48.9 | 21.3 | 46.1 | 10.0 | 56.1 | 25.5 | 53.5 | 12.2 | 45.1 | 14.0 | 36.3 | 6.2 | 77.1 | 57.2 | 72.0 | 36.1 | 62.9 |
| Ancient-Chat-LLM-7B [51] | 25.9 | 8.2 | 23.7 | 0.1 | 25.4 | 9.1 | 23.8 | 0.0 | 33.5 | 8.8 | 27.7 | 0.0 | 54.3 | 42.4 | 51.7 | 0.3 | 27.0 |
| Bloom-7B-Chunhua [51] | 49.4 | 21.4 | 47.4 | 5.6 | 48.1 | 22.0 | 46.9 | 3.5 | 51.1 | 19.6 | 47.6 | 7.1 | 76.4 | 54.6 | 72.2 | 32.9 | 48.8 |
| Xunzi-Qwen1.5-7B [47] | 62.7 | 34.9 | 60.7 | 22.5 | 65.7 | 39.8 | 64.0 | 25.9 | 49.7 | 20.0 | 46.2 | 9.5 | 76.1 | 53.1 | 71.8 | 31.6 | 69.8 |

# E   Related Work on Ancient Language Evaluation Datasets

In the process of exploring ancient Chinese evaluation benchmarks, we discovered a series of benchmark studies targeting other ancient languages. The AGREE Benchmark [70] focuses on evaluating distributional semantic models for Ancient Greek, the LogogramNLP benchmark [71] makes natural language processing analysis of ancient pictographic languages possible for the first time, the Sanskrit Word Segmentation Dataset [72] provides a foundation for syntactic and semantic processing of Sanskrit literature, and The York-Toronto-Helsinki Parsed Corpus of Old English Prose (YCOE) [73] is a 1.5 million-word syntactically annotated corpus of old English prose, based on the Toronto Corpus of old English, encompassing all major old English prose works. Compared to these benchmarks that include a small number of tasks, WenMind includes a rich variety of ancient Chinese tasks, making it possible to comprehensively evaluate the ancient Chinese capabilities of LLMs.

# F   Statement of Responsibility

The WenMind benchmark is released under the CC-BY-NC-SA-4.0 license and strictly adheres to the agreements of the original data sources. The licenses for the original data sources are detailed in Appendix B.3. We have reviewed the ethical guidelines and ensured that the content of the paper and the benchmark are in compliance with the guidelines.

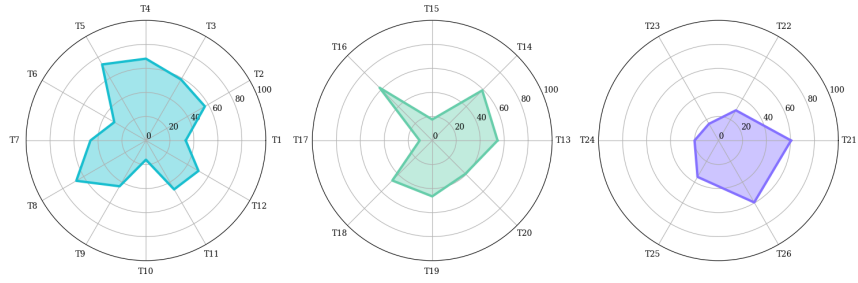

Figure 20: Performance of Baichuan2-7B-Chat.

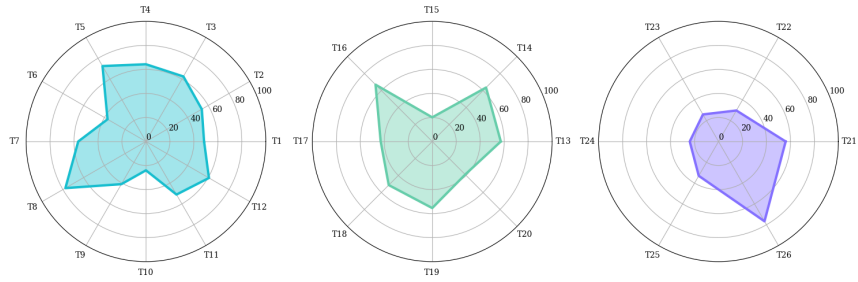

Figure 21: Performance of Baichuan2-13B-Chat.

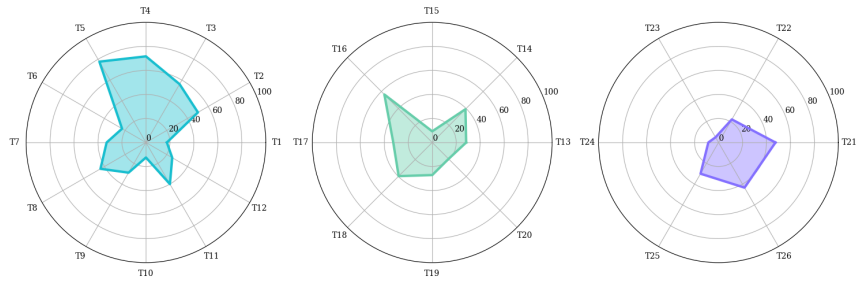

Figure 49: Performance of Bloom-7B-Chunhua.

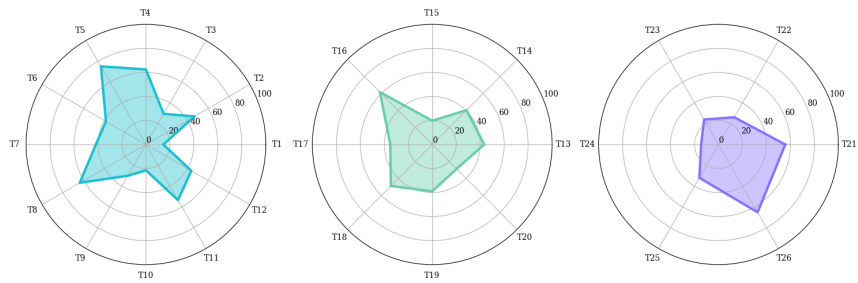

Figure 22: Performance of Firefly-Baichuan2-13B.

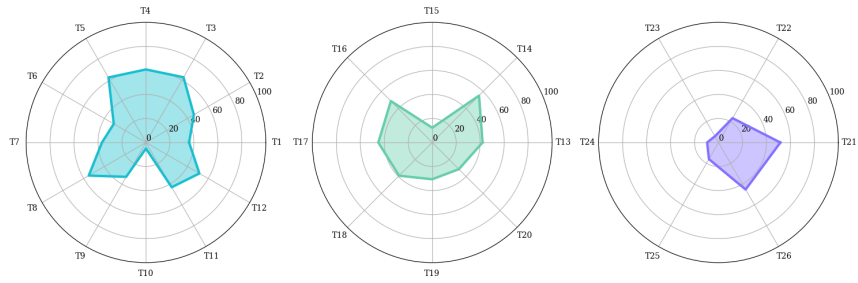

Figure 23: Performance of ChatGLM2-6B.

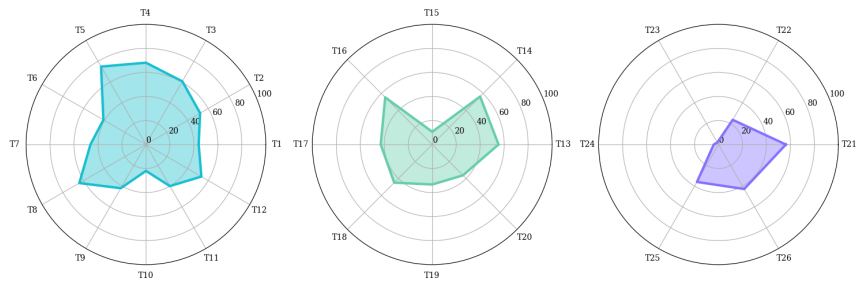

Figure 24: Performance of ChatGLM3-6B.

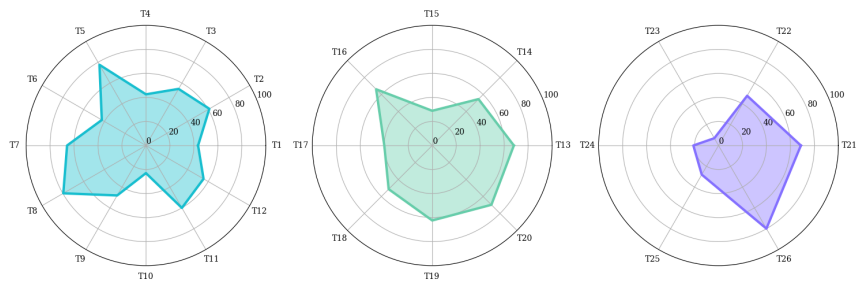

Figure 25: Performance of InternLM2-Chat-7B.

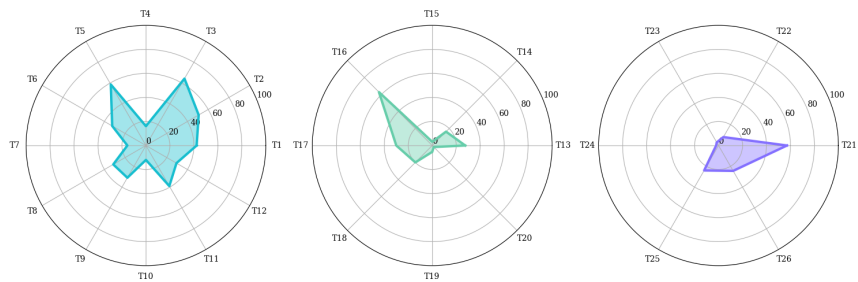

Figure 26: Performance of Qwen1.5-0.5B-Chat.

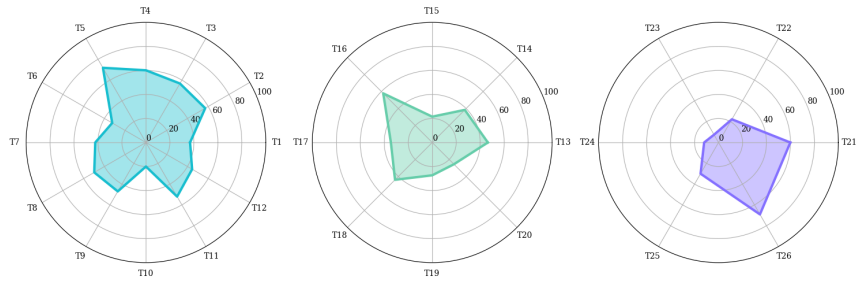

Figure 27: Performance of Qwen1.5-4B-Chat.

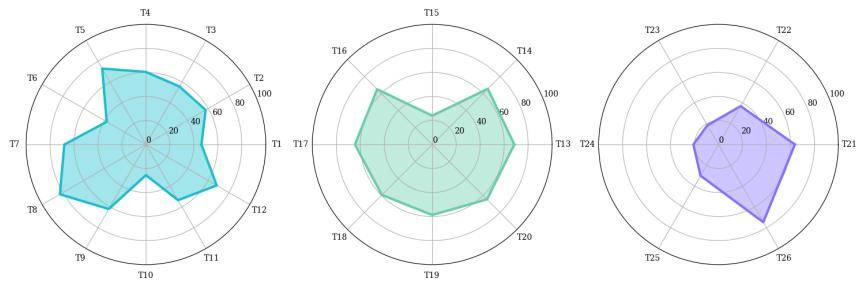

Figure 28: Performance of Qwen1.5-7B-Chat.

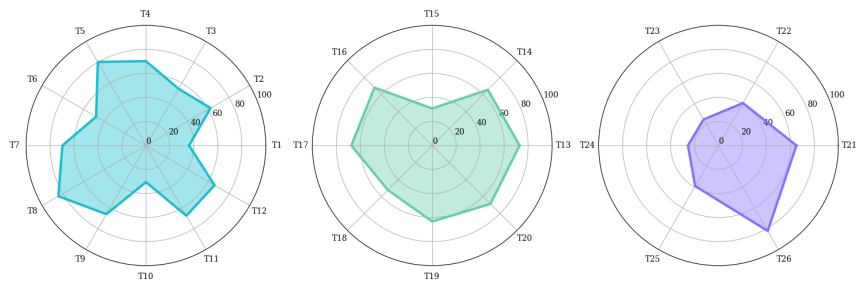

Figure 29: Performance of Qwen1.5-14B-Chat.

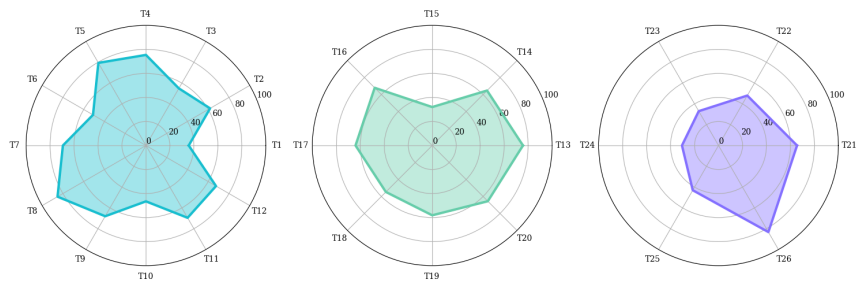

Figure 30: Performance of Qwen1.5-32B-Chat.

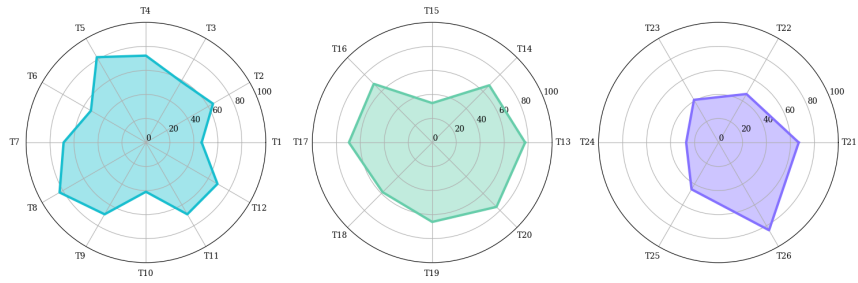

Figure 31: Performance of Qwen1.5-72B-Chat.

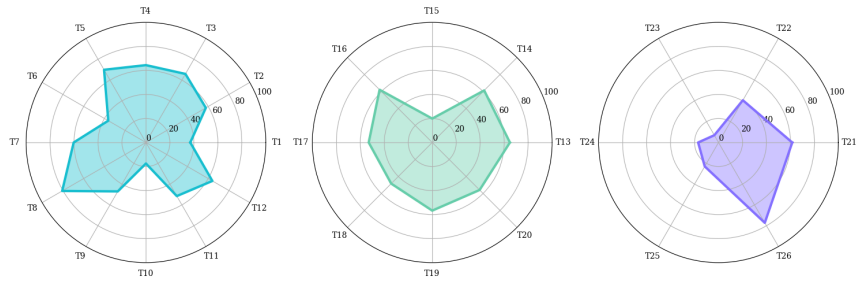

Figure 32: Performance of Yi-1.5-6B-Chat.

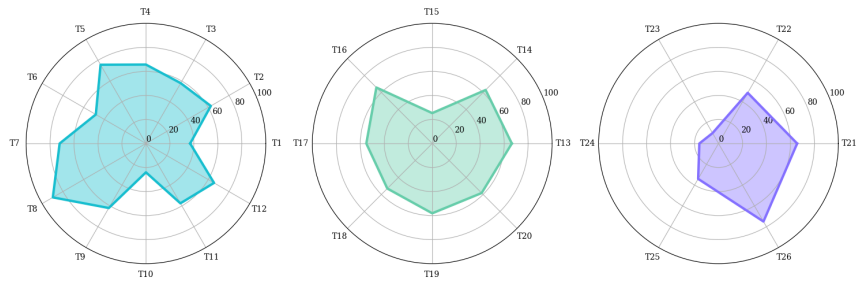

Figure 33: Performance of Yi-1.5-9B-Chat.

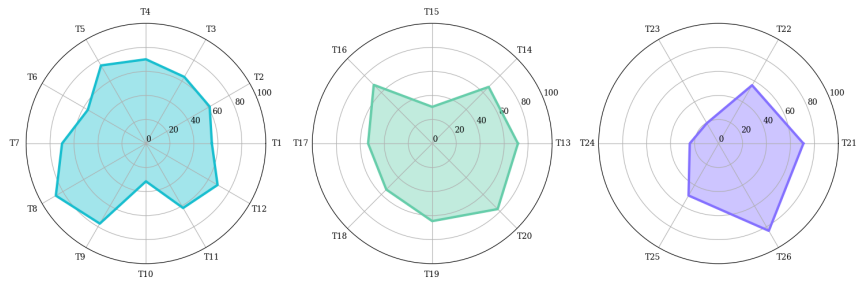

Figure 34: Performance of Yi-1.5-34B-Chat.

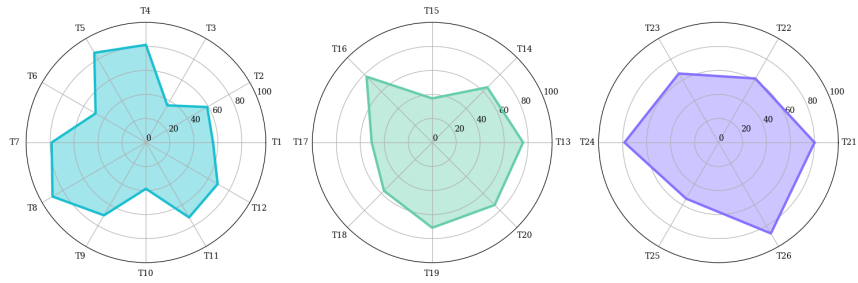

Figure 35: Performance of ERNIE-3.5-8K-0329.

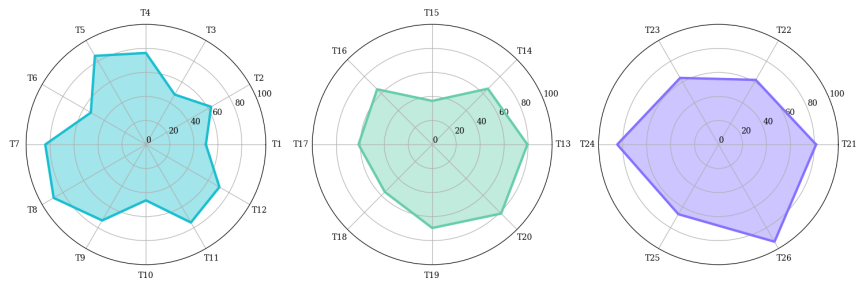

Figure 36: Performance of ERNIE-4.0-8K-0329.

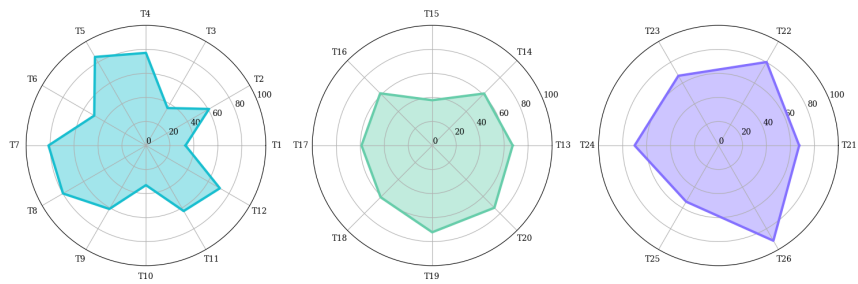

Figure 37: Performance of Spark-3.5.

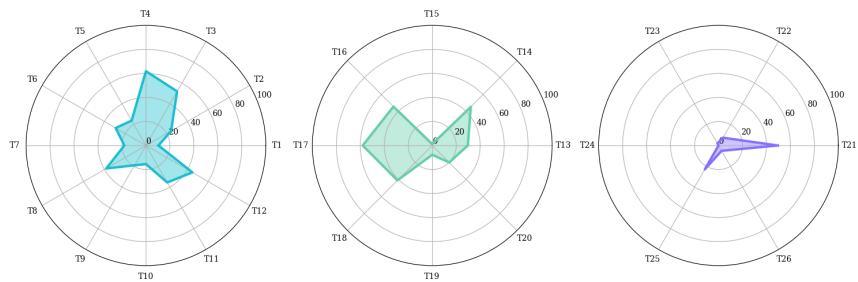

Figure 38: Performance of Gemma-1.1-7B-IT.

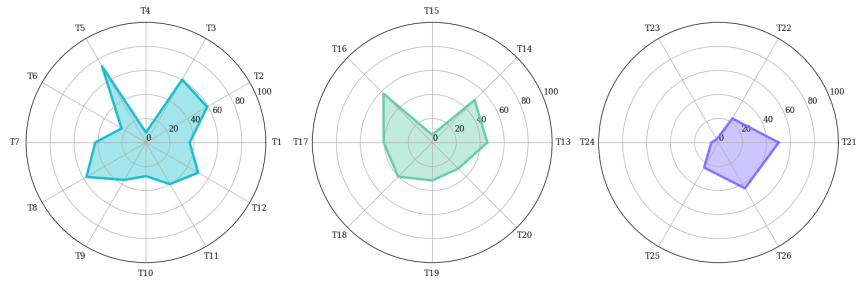

Figure 39: Performance of Ziya-LLaMA-13B-v1.1.

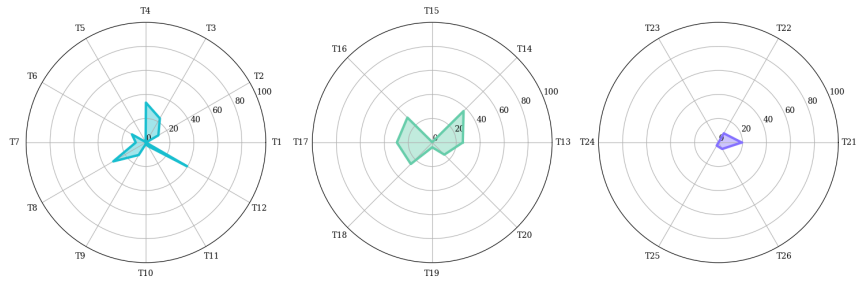

Figure 40: Performance of LLaMA2-7B-Chat.

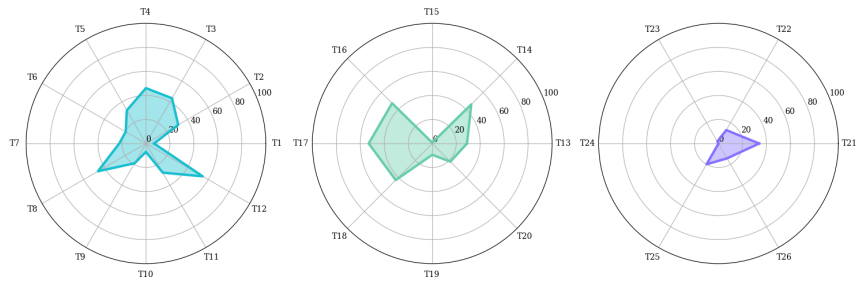

Figure 41: Performance of LLaMA2-13B-Chat.

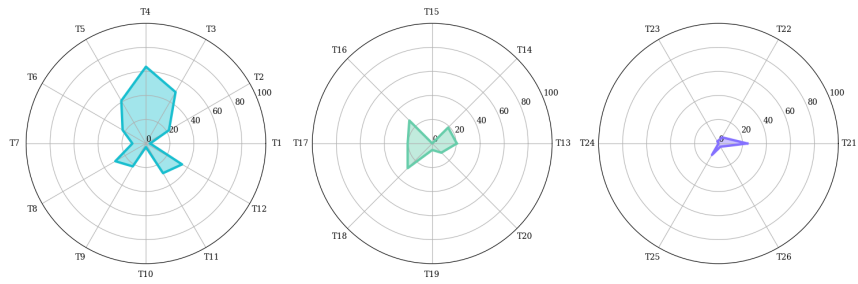

Figure 42: Performance of LLaMA2-Chinese-7B-Chat.

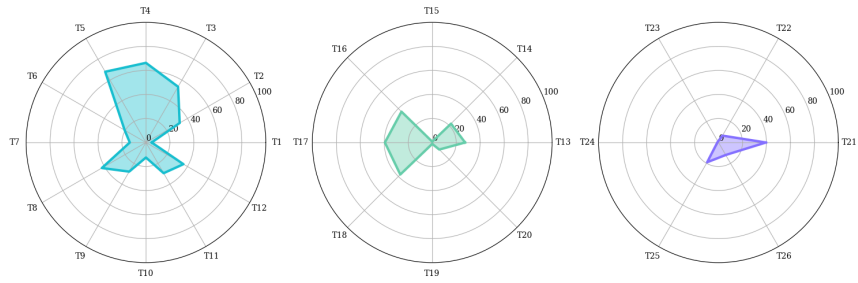

Figure 43: Performance of LLaMA2-Chinese-13B-Chat.

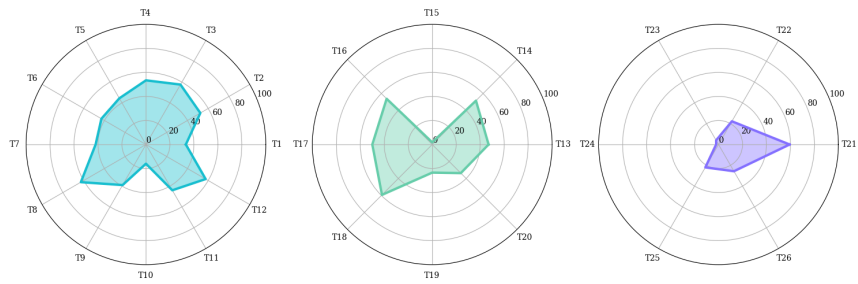

Figure 44: Performance of LLaMA3-8B-Instruct.

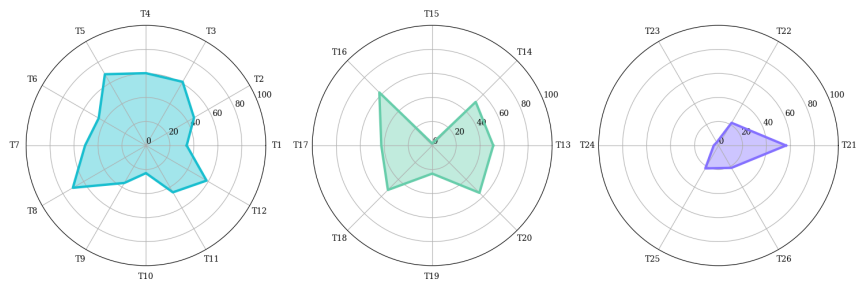

Figure 45: Performance of LLaMA3-Chinese-8B-Chat.

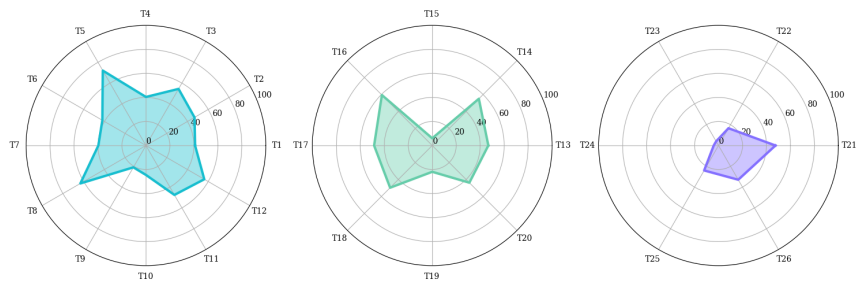

Figure 46: Performance of GPT-3.5.

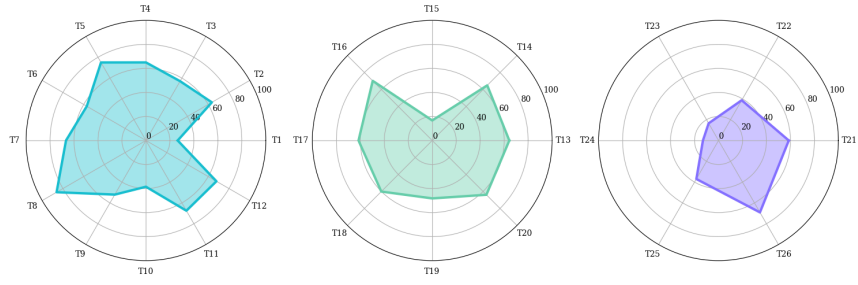

Figure 47: Performance of GPT-4.

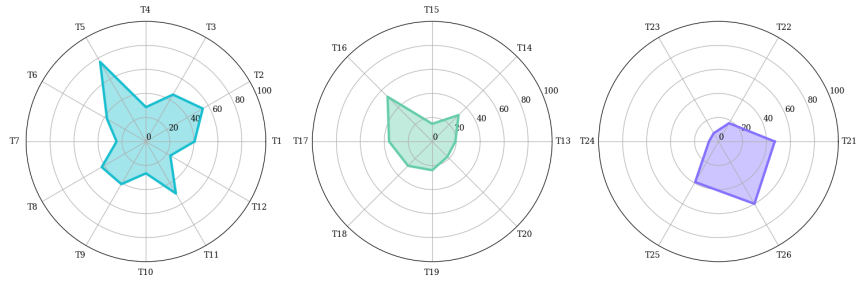

Figure 48: Performance of Ancient-Chat-LLM-7B.

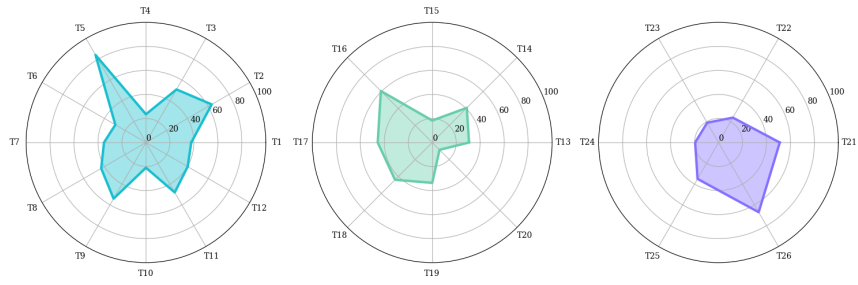

Figure 50: Performance of Xunzi-Qwen1.5-7B.

